# Uncertainty-aware Fine-tuning of Segmentation Foundation Models

**Kangning Liu**[1,2]   **Brian Price**[2]   **Jason Kuen**[2]   **Yifei Fan**[2]   **Zijun Wei**[2]   **Luis Figueroa**[2]
**Krzysztof J. Geras** [1]   **Carlos Fernandez-Granda**[1]
[1]New York University   [2]Adobe

## Abstract

The Segment Anything Model (SAM) is a large-scale foundation model that has revolutionized segmentation methodology. Despite its impressive generalization ability, the segmentation accuracy of SAM on images with intricate structures is often unsatisfactory. Recent works have proposed lightweight fine-tuning using high-quality annotated data to improve accuracy on such images. However, here we provide extensive empirical evidence that this strategy leads to forgetting how to "segment anything": these models lose the original generalization abilities of SAM in the sense that they perform worse for segmentation tasks not represented in the annotated fine-tuning set. To improve performance without forgetting, we introduce a novel framework that combines high-quality annotated data with a large unlabeled dataset. The framework relies on two methodological innovations. First, we quantify the uncertainty in the SAM pseudo labels associated with the unlabeled data and leverage it to perform uncertainty-aware fine-tuning. Second, we encode the type of segmentation task associated with each training example using a *task prompt* to reduce ambiguity. We evaluated the proposed Segmentation with Uncertainty Model (SUM) on a diverse test set consisting of 14 public benchmarks, where it achieves state-of-the-art results. Notably, our method consistently surpasses SAM by 3-6 points in mean IoU and 4-7 in mean boundary IoU across point-prompt interactive segmentation rounds. Code is available at `https://github.com/Kangningthu/SUM`

## 1   Introduction

The Segment Anything Model (SAM) [1] is a ground-breaking vision foundation model to perform interactive binary segmentation, where the goal is to identify the pixels belonging to a target region in an input image. The task is interactive, because the model accepts prompts from the user, which specify the desired target region. These prompts are typically in the form of a set of points that should be included or excluded from the target region, or a box or mask that encompasses it.

SAM is trained via a boot-strapping training procedure called a *data engine* on massive dataset of unlabeled images. The key idea is to leverage model-generated pseudo-labels to annotate the unlabeled data and use the pseudo-labels to further train the model. SAM is notable for its impressive general-purpose capabilities, allowing it to segment a wide range of objects, parts, and visual structures. Yet, the model encounters difficulties in achieving **high-quality segmentation** of intricate structures and often produces imprecise segmentation boundaries. This is problematic for applications such as image editing and automatic image annotation [2, 3], where accurate fine-grained predictions are essential.

Our analysis in Appendix I reveals that **inaccuracies within pseudo-labels** are substantial, especially in boundary regions. It has been reported in [1] that simply scaling up the amount of pseudo-labeled data by 10x only achieves marginal improvement (around 0.5 mIoU). To improve segmentation

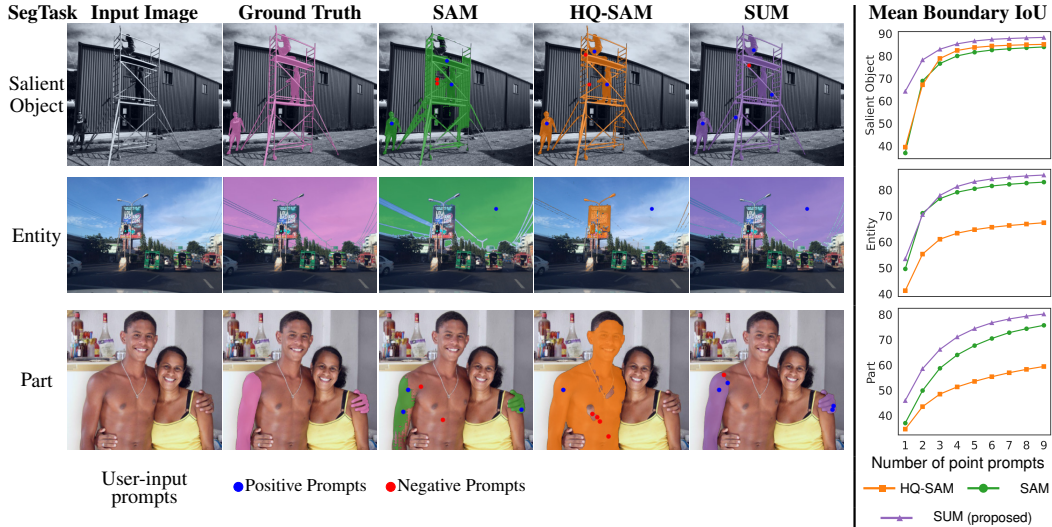

Figure 1: **SUM improves SAM without forgetting to "segment anything."** Performance comparison between the Segment Anything Model (SAM), HQ-SAM model fine-tuned on the HQ-Seg-44K dataset, and the proposed Segment with Uncertainty Model (SUM) fine-tuned using a high-quality annotated dataset and unlabeled data. Each row corresponds to held-out datasets focusing on different segmentation tasks (SegTask). **Left**: Both HQ-SAM and SUM show qualitative improvements over SAM, particularly in salient-object segmentation of complex structures (top row). HQ-SAM, however, struggles with background entities (middle row) and part segmentation (bottom row), often erroneously prioritizing objects in the foreground or entire objects. **Right**: SUM consistently outperforms SAM and HQ-SAM in quantitative comparisons, achieving the highest mean boundary IoU across diverse evaluation sets and interactive segmentation rounds. We adhere to the same point prompt sampling evaluation strategy as SAM [1] (Appendix B.3). More examples are in Appendix H.2, H.3.

quality, recent works [4, 5] suggest fine-tuning SAM exclusively with data paired with high-quality human annotation. A key challenge is that the size of those datasets is usually relatively small, and the diversity of images and annotations they contain is limited. Consequently, the **model may overfit** the characteristics of the fine-tuning dataset, which degrades its performance in other situations (e.g., segmentation tasks different from the fine-tuning dataset). Currently, the most common strategy to mitigate this is lightweight finetuning, which only updates a small subset of model parameters [4, 5, 6, 7, 8]. However, our experiments find that the overfitting problem still persists, as demonstrated in Fig. 1 and Section 4.2.

In this paper, our goal is to investigate how to better learn from pseudo-labels, with the help of high-quality human-annotated data. We then apply these insights to **improve the accuracy of SAM without compromising its generalization**. Our proposed framework, the Segmentation with Uncertainty Model (SUM), performs fine-tuning in a manner that is aware of the regions where model-generated pseudo-labels *have questionable accuracy*. This is achieved by explicitly estimating pseudo-label uncertainty in the unlabeled training data, in the form of an *uncertainty map* generated by a specialized module. The uncertainty map is then incorporated into a novel *uncertainty-aware training loss*, which reduces the influence of pseudo-label regions that are expected to be inaccurate. The framework has an additional advantage when training interactive-segmentation models. During training, user prompts are typically generated randomly based on the available labels. Our framework additionally enables *uncertainty-aware prompt sampling*, which avoids producing misleading prompt locations.

Our framework also addresses an additional challenge in fine-tuning segmentation foundation models. Human annotations usually correspond to specific segmentation tasks. For instance, the HQ-Seg-44K [4] used by previous fine-tuning approaches [4] focuses on *salient-object segmentation*, where the target regions may include multiple entities in a complex arrangement. In contrast, current general-purpose interactive segmentation models, such as SAM, are mainly trained for *entity* or *part* segmentation, where target regions are associated with individual entities or object parts, respectively. To take this into account, we propose a novel *task prompt*, which informs the model of the desired

segmentation task during training. This resolves ambiguities between the human and model-generated labels, and can also be leveraged to specify the desired segmentation task during inference.

To evaluate SUM, we performed systematic experiments with diverse segmentation datasets, resulting in the following insights. (1) Existing fine-tuning approaches forget to *segment anything*. After fine-tuning on HQ-Seg-44K, they outperform SAM on salient-object segmentation but underperform on held-out datasets where annotations are associated with entities or object parts (Fig. 1 and Section 4.2). (2) Including unlabeled data associated with SAM pseudo-labels in the fine-tuning dataset prevents forgetting, but also yields a limited gain in performance if noise inherent in pseudo-labels is not effectively managed (see Section 4.3). (3) SUM is able to effectively leverage the labeled and unlabeled data to achieve superior performance (see Section 4.2 and Appendix G.1). This is demonstrated in controlled experiments using publicly available fine-tuning datasets, as well as on a large-scale internal dataset. An ablation study (see Section 4.3) shows that the different methodological innovations (uncertainty-based loss, uncertainty-based prompting, and task prompting) all contribute to this gain in performance.

## 2   Related Work

**Segmentation Foundation Models**   Segmentation is a fundamental task in computer vision, which includes semantic segmentation [9, 10, 11, 12], instance segmentation [13, 14], panoptic segmentation [15, 16], and interactive segmentation [17, 18, 19, 20]. Inspired by the breakthroughs of language foundation models [21, 22, 23, 24], the field has recently witnessed the emergence of large-scale segmentation models pre-trained on massive datasets, which demonstrate remarkable generalization capabilities. Painter [25] and SegGPT [26] proposed robust in-context learning frameworks enabling segmentation of images given image-mask prompts. Recently, SEEM [27] introduced a multi-modal reference-prompted general segmentation model. Our work builds on the Segment Anything Model (SAM) [1], a ground-breaking foundational model for interactive segmentation. SAM is trained leveraging a boot-strapping training procedure called a *data engine*, which encompasses multiple phases with different degrees of human supervision, culminating in a fully-automated stage where the pseudo-labels are generated automatically from the model.

**Fine-tuning Segmentation Foundation Models**   Different methods have been proposed to extend SAM, and other foundation models, in different ways, such as increasing inference speed [28, 29], performing all-purpose matching [30] or personalized segmentation [31], increasing granularity [32], enabling language-informed segmentation guidance [33], and adapting to specific domains (e.g., medical imaging [34, 35, 6, 7, 36, 37, 38, 39, 40, 41], agriculture [8], remote sensing [42], shadow detection [43, 44, 45], or water leakage [46]).

Our goal is to enhance SAM's accuracy while maintaining its generalizability. HQ-SAM [4] pursues a similar goal, but primarily focuses on minimal architectural adaptations for light-weight fine-tuning on high-quality human-annotated data. Here, we focus on an effective framework for leveraging limited human-annotated data and diverse unlabeled data.

**Semi-supervised Semantic Segmentation**   Semi-supervised semantic segmentation [47, 48, 49, 50] also aims to leverage unlabeled data when labeled data is limited. However, the purpose is usually to enhance the performance of a model on a specific task. In this work, we aim to instead leverage the unlabeled data to preserve the generalization ability of a foundation model for diverse segmentation tasks (e.g. part, object, background), which may not be represented in the labeled data. The majority of semi-supervised approaches enforce consistency between predictions corresponding to augmented versions of an unlabeled image. This is achieved through various specialized architectures such as teacher-student networks [51, 52, 53], parallel networks [54, 55], or incorporating additional representation heads [56]. In contrast, our proposed method seeks to improve the pseudo-labels based on the data engine of SAM, requiring minimal modifications to the existing architecture. Combining these two types of approaches is an interesting direction for future investigation.

**Uncertainty-aware Segmentation**   Uncertainty-aware segmentation has been explored for semi-supervised settings (e.g. U2PL [51]), and domain-adaptive segmentation [57, 58, 59]. These methods typically employ prediction confidence [51, 57] or discrepancy among multiple prediction heads [59, 58] to quantify uncertainty. Our approach differs fundamentally from previous approaches in both

the generation of uncertainty maps and the utilization of uncertainty maps. SAM undergoes multiple rounds of self-training, which leads to the accumulation and overfitting of errors in pseudo-labels [60]. During our evaluation, the model frequently predicts incorrect regions with high confidence, and different heads agree on these erroneous areas. Experiments detailed in Section 4.2 show that previous approaches are ineffective in our context. Our main insight is to use external supervision to identify and correct systematic biases that accumulate during training. Further discussion can be found in Appendix E.2.

## 3 Method

### 3.1 Preliminary Background: Segment Anything Model (SAM)

The Segment Anything Model (SAM) [1] consists of three modules: an image encoder based on a large vision transformer, a lightweight prompt encoder for geometric inputs (e.g., points, boxes, mask indicating the target in interactive segmentation), and a lightweight mask decoder. The decoder combines the outputs of both encoders to produce a predicted segmentation mask.

During training, the model is provided interactive prompts iteratively. The first prompt is obtained by randomly selecting a point or bounding box from the target mask. Then, subsequent point prompts are randomly sampled within the error region, which is the discrepancies between the target mask and prior model prediction. At each training step, the model receives the preceding mask prediction as an additional input.

### 3.2 Segment with Uncertainty Model (SUM)

The Segment with Uncertainty Model (SUM) is a fine-tuning framework to improve the segmentation quality of an existing foundational segmentation model, which builds upon the Segment Anything Model (SAM). A diagram illustrating the framework is provided in Fig. 2. The framework assumes the availability of a *human-labeled set* with high-quality labels (e.g. HQ-Seg-44K), and a large, diverse *unlabeled set* (e.g. SA-1B [1]). These datasets are used to fine-tune SAM via interactive segmentation training. Training is performed with human annotations for the labeled data and with SAM-generated pseudo-labels for unlabeled data. These pseudo-labels are produced in accordance with SAM's automatic mask generation pipeline (for more details, see Appendix B).

SUM contains several novel elements designed to improve the fine-tuning process. The pseudo-labels generated by SAM are inaccurate, which is problematic for training. SUM addresses this by exploiting the insight that this inaccuracy is systematic and *predictable*. Uncertainty maps are leveraged to modify the fine-tuning cost function and the generation of training prompts for the unlabeled examples to make them uncertainty aware, as explained in Sections 3.2.1 and 3.2.2. The uncertainty maps are produced by an uncertainty-quantification module described in Section 3.2.3. Additionally, SUM incorporates a task prompt, described in Section 3.2.4, which allows us to specify the desired segmentation task.

#### 3.2.1 Uncertainty-aware Loss

SUM combines human-annotated and unlabeled data, utilizing different training losses for each. For the human-annotated data, SUM minimizes the same training loss as SAM: a linear combination of focal loss [61] and Dice loss [62]. For the unlabeled data, we propose to utilize uncertainty-aware versions of both losses instead, which are then combined in the same way as SAM. The goal is to down-weight the influence of uncertain areas in the pseudo-labels and increase the influence of regions with high confidence.

The *uncertainty-aware focal loss* $l_{\text{UAfocal}}$ is obtained by incorporating pixel-specific weights to the standard focal loss [61]. Let $\hat{y}_i$ denote the binary pseudo-label corresponding to the $i$th pixel in an input image, $p_i$ the corresponding probability predicted by the model, and $u_i$ the corresponding entry of the uncertainty map obtained from the SUM uncertainty-quantification module (see Section 3.2.3). The uncertainty is between 0 (highly certain) and 1 (highly uncertain). The contribution of each pixel to the focal loss $l_{\text{focal}}(p_i, \hat{y}_i)$ is weighted by $c_i = 1 - u_i$, which can be interpreted as the certainty

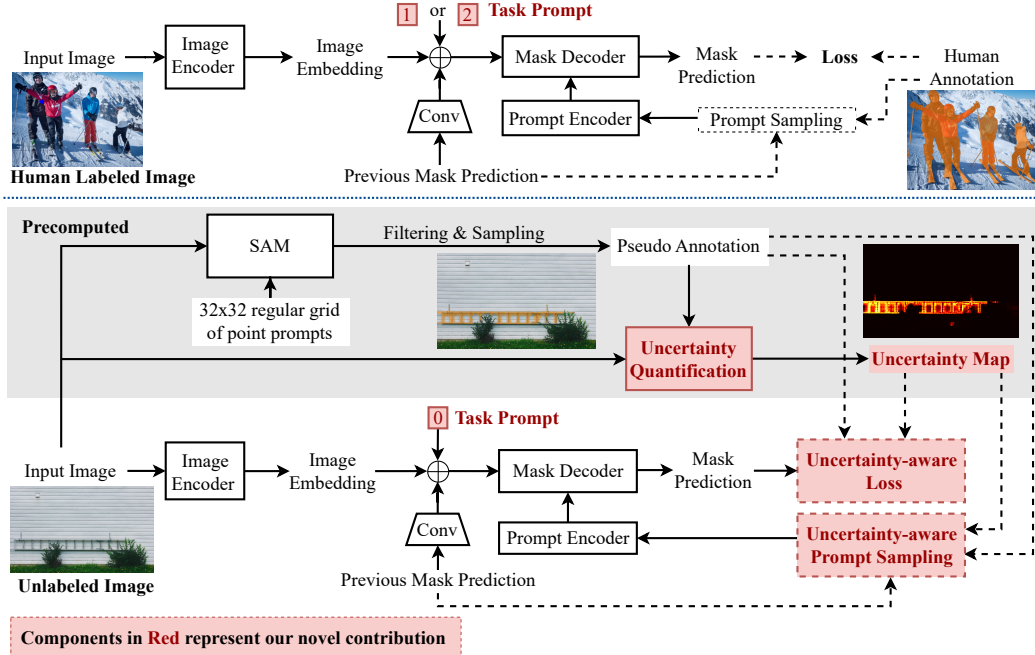

Figure 2: **Segment with Uncertainty Model (SUM) framework.** The diagram describes the proposed SUM framework, which utilizes both human-annotated and unlabeled data to fine-tune a foundation model for interactive segmentation. **Top**: When processing human-annotated examples, interactive prompts are sampled based on the binary-mask labels and fed iteratively into the model along with the image. Since this binary mask depends on the type of segmentation task desired by the user, SUM incorporates a task prompt that specifies the task relevant to each annotation (1 for salient-object segmentation and 2 for entity segmentation). **Bottom**: For unlabeled images, the iterative prompts are sampled based on model-generated binary pseudo-labels, which may be inaccurate. SUM includes an uncertainty-quantification module that processes the pseudo-labels, generating an uncertainty map. This map is leveraged within an uncertainty-aware loss function used for training, and also informs how the interactive prompts are sampled. For all unlabeled data, the task prompt is set to 0. Components highlighted in red represent our novel contributions, whereas the remaining components adhere to the SAM framework.

associated with that pixel:

$$l_{\text{UAfocal}}(p, \hat{y}) = \frac{1}{m} \sum_{i=1}^{m} c_i \cdot l_{\text{focal}}(p_i, \hat{y}_i) \tag{1}$$

The *uncertainty-aware Dice loss* $l_{\text{UADice}}$ is an adaptation of the standard Dice loss, which only preserves pixels where the certainty is above a certain threshold $\theta$ (see Appendix F for a discussion on the choice of this hyperparameter):

$$
\begin{aligned}
l_{\text{UADice}}(p, \hat{y}) &= 1 - \frac{2 \cdot I_{\text{UA}}(p, \hat{y}, u) + \varepsilon}{S_{\text{UA}}(p, \hat{y}, u) + \varepsilon} \\
I_{\text{UA}}(p, \hat{y}, u) &= \sum_{i=1}^{m} \hat{y}_i \cdot p_i \cdot \mathbb{I}\left[c_i > \theta\right], S_{\text{UA}}(p, \hat{y}, u) = \sum_{i=1}^{m} (\hat{y}_i + p_i) \cdot \mathbb{I}\left[c_i > \theta\right],
\end{aligned}
\tag{2}
$$

where $\epsilon$ is a small positive constant to ensure numerical stability. $I_{\text{UA}}$ and $S_{\text{UA}}$ denote the filtered intersection and sum, respectively, of the predicted probabilities $p$ and pseudo-labels $\hat{y}$.

### 3.2.2 Uncertainty-aware Prompt Sampling

SUM applies the same prompt-sampling strategy as SAM for the human-annotated data during interactive training. The first prompt is obtained by selecting a point or bounding box at random from the target mask containing the positive segmentation labels. Then, subsequent *positive* point prompts

are selected from regions in the target mask not present in the predicted mask, and *negative* point prompts are selected from regions in the predicted mask not present in the target mask. When the target mask is inaccurate, which is often the case when it relies on pseudo-labels, this is problematic because the point prompts may be completely incorrect. To address this, SUM leverages the uncertainty maps generated by the uncertainty-quantification module (see Section 3.2.3) to perform uncertainty-aware prompt sampling. Let $\mathcal{R}$ denote the region used by SAM for prompt sampling.[1] Uncertainty-aware sampling selects each pixel $i$ in $\mathcal{R}$ with probability $P_i = c_i / \sum_{j=1}^{n} c_j$, where $c_i$ is defined above as the certainty associated with the pixel. This strategy ensures that prompts are sampled mostly from regions with high confidence.

### 3.2.3 Generation of Uncertainty Maps

In this section, we describe how uncertainty quantification is performed within the SUM framework, detailing the generation of uncertainty maps associated with pseudo-labels. For this purpose, SUM uses a specialized mask-refinement module. The module receives the logits produced by the SAM model[2] as an input and outputs a refined prediction of the segmentation mask. The uncertainty map is calculated by taking the absolute difference between the refined prediction and the sigmoid-transformed probabilities of the SAM logits. The uncertainty value for the $i$-th pixel in the uncertainty map $u_i$ is given by:

$$u = |\sigma(s_i) - \sigma(r_i)| \tag{3}$$

where $\sigma$ is the sigmoid function, $s_i$ denotes the SAM logits for the $i$-th pixel, and $r_i$ denotes the refined prediction for the $i$-th pixel. This yields values between 0 (no difference, indicating low uncertainty) and 1 (large difference, indicating high uncertainty). This is illustrated in Fig. 3. The mask-refinement module also receives as input a box prompt encompassing the pseudo-label and the point prompt used to generate the pseudo-labels.

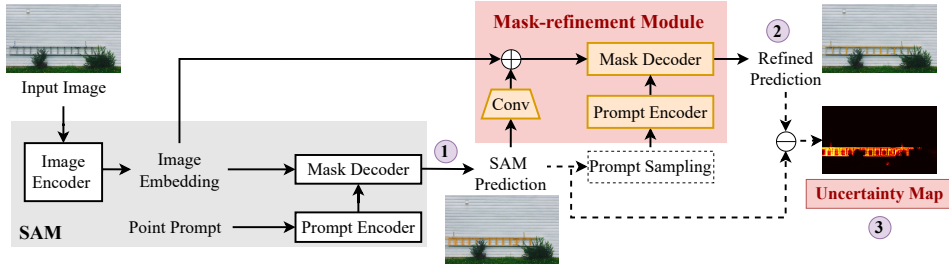

Figure 3: **Generation of uncertainty maps**. (1) The mask-refinement module receives as input the segmentation prediction produced by SAM. (2) The module produces a refined segmentation mask. (3) The uncertainty map equals the absolute difference between the SAM and refined predictions. Additional visual examples of the uncertainty map are presented in Appendix Fig. 10.

The mask-refinement module is implemented by fine-tuning SAM's decoder and prompt encoder. To be clear, these fine-tuned components are exclusively used to generate uncertainty maps before training SUM. We train the mask-refinement module to refine the SAM pseudo-label masks using the corresponding high-quality labels in the high-quality human-annotated dataset. During training, we feed the SAM pseudo-label logits to the mask-refinement module in the same way that SAM incorporates previous mask predictions in its interactive segmentation training protocol.

Appendix D provides an in-depth description of the design of the mask-refinement module, including quantitative and human-subject evaluations of the resulting segmentation maps, the relationship between uncertainty and pseudo-label accuracy, alongside a comparative analysis of model variants.

Although it is possible to replace SAM pseudo-labels with refined masks in a naive manner, we empirically find that it less effective than uncertainty-aware training, as reported in Table 2. This suggests that potential errors in mask refinement (see Appendix Fig. 9) may impact performance.

### 3.2.4 Task Prompt

Segmentation datasets often focus on different segmentations *tasks*, resulting in dramatic and systematic differences between their labels. For example, the HQ-Seg-44K [4] dataset targets *salient-object segmentation*, where a segmentation mask may include multiple entities in a complex arrangement. In contrast, the pseudo-labels produced by SAM for the SA-1B dataset mostly correspond to *entity segmentation* (where masks are associated with individual entities) or *part segmentation* (where masks indicate parts of objects). This is problematic for joint training, as it introduces a high degree of ambiguity regarding the segmentation mask that the model should predict after the initial prompt.

To resolve this ambiguity, SUM includes a task prompt to accompany the first point or box prompt, indicating the segmentation task associated with each training example. In practice, we identify the task based on the dataset associated with each example. We assign 0 to examples from the unlabeled dataset, 1 to examples from human-annotated datasets focused on salient-object segmentation, and 2 to examples from human-annotated datasets focused on entity segmentation. See Appendix H.1 for an illustration. More categories can be added to account for additional segmentation tasks (e.g., 3 for part segmentation). The task prompt is implemented via an embedding layer, which acts as a lookup table mapping each discrete task identifier to a unique high-dimensional vector embedding learned during training. This task embedding is combined via element-wise addition with the embedding produced by the SAM image encoder, as illustrated by Fig. 2.

## 4 Experiments

### 4.1 Experiment Settings

**Training Sets** We utilize multiple training sets built using the following datasets:

- **SA-250K**: An unlabeled subset extracted from SA-1B dataset [1] containing 250,000 images.
- **HQSeg-44K** [4]: A human-annotated set of 44,320 images with high-quality salient-object masks.
- **EntitySeg Training Set** [63]: A human-annotated set of 31,913 images, each with an average of 20 entity masks, detailing both foreground and background.
- **Internal Dataset**: A human-annotated set containing 252,798 images with salient-object masks, 60,798 images with entity masks, and 153,046 images focused on human parsing. More details are provided in Appendix C.

Based on these datasets, we define three fine-tuning (FT) sets with varying budgets of human-annotated data: (1) *FT-Small*: SA-250K and HQSeg-44K, enabling a fair comparison with existing fine-tuning methods such as HQ-SAM [4]; (2) *FT-Medium*: SA-250K, HQSeg-44K and EntitySeg; (3) SA-250K and the internal dataset.

**Evaluation Sets** All the models are evaluated on the same collection of test sets, designed to assess performance on three different segmentation tasks:

- **Salient Object**: COIFT [64], validation sets of DIS [65] and ThinObject5K [65], HR-SOD [66], VOCEVAL [67], and BIG [67]. These datasets contain object-centric images with salient objects and binary masks. The training sets in DIS-VD and ThinObject5K are included in the HQSeg-44K dataset. COIFT, HRSOD, VOCEVAL, and BIG are used for zero-shot evaluation.
- **Entity**: Three EntitySeg [63] validation sets with 454, 459 and 401 images. These sets feature complex scene images with multiple entities ranging from foreground instances to background regions (e.g., sky, road).
- **Part**: Three human/clothing parsing datasets: the validation set of Fashionpedia [68] containing 1,148 images (861 of which have subpart masks), the first 1,000 images of the validation sets of Multi-Human Parsing [69] and Easyportrait [70]. We also include the first 1,000 images of the validation set of Paco [71], which contains a wider variety of object types.

**Evaluation** 1) **Metrics**: We use standard metrics to evaluate segmentation quality: mean Intersection over Union (mIoU) and mean boundary IoU (mBIoU). We aggregate these metrics via averaging for the test datasets in each category. Also, an overall average is derived by aggregating results from all test datasets. 2) **Test prompts**: Unless specified otherwise, we generate test prompts using the

center point sampling technique described in SAM [1]. We also report box-prompt segmentation results in Appendix G. 3) **Comparison to single-output model**: SAM and SUM generate multiple masks when a single prompt is provided. For evaluation, we generally select the output mask that most closely aligns with the provided ground truth. However, when comparing it to HQ-SAM, which produces a single output, we use a single output fixed in advance.

**Baselines** We compare the proposed method SUM against the following baselines: (1) *Human annotation only*, a method where fine-tuning is exclusively performed with human-labeled data; and (2) *Vanilla*, where fine-tuning is performed on unlabeled and human-annotated data following the standard SAM training protocol, without discriminating between human and SAM pseudo labels.

**Implementation Details:** SUM is implemented by fine-tuning a SAM model based on a ViT-H backbone. The experiments were done adhering to SAM's configuration for loss weighting. Key training parameters, including learning rate, duration, hyperparameters, and schedules for both human-annotated and unlabeled data, are detailed in Appendix B. Although the SUM framework introduces additional steps to generate uncertainty maps, these steps remain manageable within practical constraints. A detailed discussion on training complexity and computational overhead is provided in Appendix B.2.

## 4.2 Comparing SUM with SoTA Methods

**Comparison with HQ-SAM** Fig. 4 evaluates and compares different fine-tuning strategies, which are all constrained to utilize the same lightweight fine-tuning scheme and architecture as HQ-SAM [4] (NeurIPS 2023) for a fair comparison. We compare (1) the proposed SUM method; (2) the original HQ-SAM, fine-tuned following the Human-Annotation-Only baseline approach (i.e., fine-tuning exclusively on HQ-Seg-44k); and (3) the *vanilla* baseline, which utilizes all of FT-Small. Evaluation is carried out over several rounds of interactive segmentation. HQ-SAM excels in salient-object segmentation unsurprisingly, as it is fine-tuned on a dataset associated with this segmentation task, but its performance is worse than the other models for entity and part segmentation (and in fact also worse than SAM, as shown in Figure 1). Conversely, the *vanilla* approach, which uses unlabeled data and SAM pseudo labels, performs worse than HQ-SAM on salient-object segmentation. SUM, utilizing the same HQ-SAM lightweight structure, achieves similar performance to HQ-SAM on salient-object segmentation while clearly outperforming all other models on entity and part segmentation. Detailed dataset-wise comparisons are provided in Appendix G.3.

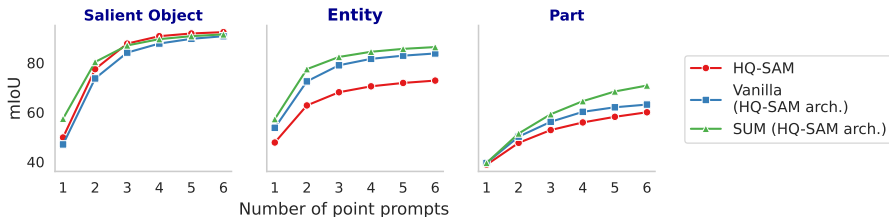

Figure 4: Comparison of HQ-SAM [4] with Vanilla and SUM fine-tuned using the same lightweight scheme as HQ-SAM. SUM matches HQ-SAM and outperforms Vanilla in salient-object segmentation and is superior in entity and part segmentation.

**Comparison with Other Light-weight Fine-tuning Strategies** The left subfigure of Fig. 5 illustrates the comparison of SUM fine-tuned on the FT-Small against various strategies applied to fine-tune SAM on the human-annotated HQSeg-44k dataset: *Decoder+Prompt-Encoder FT:* fine-tuning only the decoder and prompt encoder, *Adapter FT:* fine-tuning only the Adapter [72] plugged into the image encoder and *LoRA FT:* fine-tuning SAM with LoRA [73] in the same way as SAMed [35]. Our results suggest that fine-tuning SAM exclusively on a specific human-annotated dataset can easily result in overfitting, regardless of the fine-tuning strategy. Conversely, SUM enhances the overall performance and mostly retains SAM's generalization capabilities.

**Comparison with Semi-supervised Methods** To assess SUM's performance relative to other methods utilizing unlabeled data, we adapt AugSeg [49], a SoTA semi-supervised semantic segmentation method, to the interactive segmentation pipeline. AugSeg employs a teacher-student framework. For

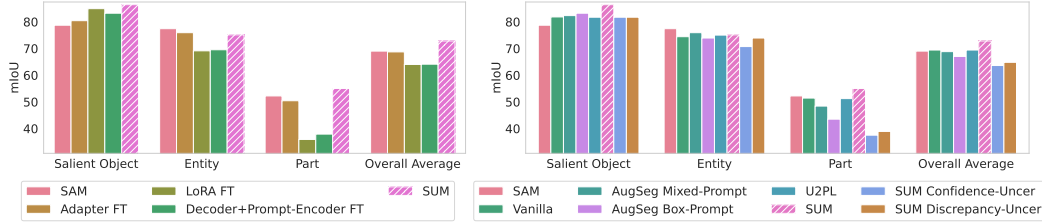

Figure 5: **Left:** Comparison of single point-prompt segmentation mIoU for SUM versus models fine-tuned using various strategies on the HQSeg-44K dataset. All competing models improve on the salient-object segmentation task associated with this dataset but deteriorate on other segmentation tasks. **Right:** mIoU for single point-prompt segmentation: comparisons of models fine-tuned on FT-Small dataset with various strategies. SUM clearly outperforms all other strategies on salient-object and part segmentation, while preserving a competitive performance on entity segmentation.

Table 1: **3-point-prompt segmentation**. This table reports 3-point-prompt interactive segmentation mean IoU and mean Boundary IoU of different models fine-tuned on FT-Small.

| Metrics | Task | SAM (without fine-tuning) | Vanilla | AugSeg Mixed-Prompt | UP2L | SUM | SUM Confidence-Uncer | SUM Discrepancy-Uncer |
|---------|------|---------|---------|---------|------|-----|------|------|
| mIoU | Salient object | 86.1 | 89.2 | 90.6 | 89.1 | **91.2** | 89.1 | 89.1 |
| | Entity | 82.7 | 81.2 | 82.9 | **83.4** | **83.4** | 82.1 | **83.4** |
| | Part | 63.2 | 61.8 | 59.9 | 64.2 | **67.2** | 59.3 | 62.4 |
| | Overall average | 77.2 | 77.7 | 78.0 | 79.0 | **81.0** | 77.0 | 78.4 |
| mBIoU | Salient object | 77.0 | 81.0 | 82.5 | 81.0 | **83.2** | 81.0 | 81.0 |
| | Entity | 76.6 | 74.6 | 77.0 | **77.2** | **77.2** | 76.0 | 76.5 |
| | Part | 59.4 | 57.5 | 55.8 | 60.2 | **63.3** | 55.2 | 58.1 |
| | Overall average | 70.6 | 71.3 | 71.8 | 72.8 | **74.8** | 70.7 | 71.9 |

the human-annotated data, we employ SAM's training procedure to train the student network, with both point and box prompts. For the unlabeled data, we apply a consistency loss based on predictions from two augmented versions of the same image, processed by the student and teacher networks. Interactive segmentation requires prompt sampled from target labels. We utilize model-generated pseudo-labels to generate the prompts for unlabeled data in two different ways: 1) AugSeg Mixed-Prompt employs a combination of point and box prompts, and 2) AugSeg Box-Prompt uses only box prompts. We provide more implementation details and discussion in Appendix E. As depicted in the right subfigure of Fig. 5, SUM clearly outperforms both versions, achieving a mIoU of 72.8, compared to 68.8 for AugSeg Mixed-Prompt, 67.0 for AugSeg Box-Prompt.

The choice of uncertainty quantification is critical for the effectiveness of pseudo-label utilization. To understand how ours compares with existing uncertainty quantification approaches, we further evaluate two variants: (1) **SUM Confidence-Uncer**, which uses logit confidence of SAM prediction to quantify pseudo-label uncertainty, similar to U2PL [51] (CVPR 2022); (2) **SUM Discrepancy-Uncer**, which quantifies uncertainty using the discrepancy between multiple predictions for a target mask, similar to Zheng et al. [58]'s work (IJCV 2021). Results in the right sub-figure of Fig. 5 indicate that these variants underperform ours significantly. We also tested an improved U2PL incorporating the same uncertainty map as SUM and task prompt training, but it still underperforms SUM. Further details are in Appendix E. We provide 3-point-prompt interactive segmentation comparison in Table 1. The performance of SUM on part segmentation improves, which is not explicitly represented in the human annotations. This demonstrates SUM's capability to **enhance accuracy across general tasks**.

**Performance of SUM under Various Annotation Budgets** In Appendix G.1, we evaluate our proposed model (SUM) when fine-tuned under various annotation budgets, utilizing the FT-Medium and FT-Large datasets. SUM achieves consistent improvements regardless of the fine-tuning budgets.

**Performance of SUM on Additional Evaluation Sets** To further validate the generalization of our proposed methods, we have expanded our evaluation by testing SUM and SAM on 8 additional evaluation sets that include a variety of image types. For reproducibility, SUM is fine-tuned on the public dataset FT-Medium. As shown in Appendix G.6, SUM consistently outperforms SAM.

## 4.3 Ablation Experiments

Table 2: **Ablation study**. This table reports interactive segmentation mean IoU of different ablated versions of SUM, showing individual gains provided by uncertainty-aware fine-tuning and task prompts. Mean Boundary IoU and detailed dataset-wise comparisons are provided in Appendix F.4.

| Metrics | Point Number | Task | SAM (without fine-tuning) | Vanilla | Mask Refinement | SUM w/o Task Prompt | SUM | SUM Continuous TP |
|---|---|---|---|---|---|---|---|---|
| mIoU | 1 | Salient object | 78.7 | 80.4 | 81.5 | 81.1 | **85.2** | 84.7 |
| | | Entity | 77.4 | 78.8 | 78.1 | 78.6 | **79.8** | 79.2 |
| | | Part | 52.2 | 50.2 | 52.3 | 54.4 | **55.3** | 54.6 |
| | | Overall average | 69.0 | 69.3 | 70.3 | 71.0 | **73.4** | 72.8 |
| | 3 | Salient object | 86.1 | 90.4 | 91.5 | 91.4 | **91.6** | 89.6 |
| | | Entity | 82.7 | 86.5 | 86.2 | **86.7** | 86.6 | 85.7 |
| | | Part | 63.2 | 64.4 | 66.7 | 67.8 | **67.9** | 67.7 |
| | | Overall average | 77.2 | 80.3 | 81.5 | 82.0 | **82.1** | 81.0 |

**Ablation on Uncertainty-aware Training and Task Prompt** Table 2 compares the Vanilla baseline and the following ablated versions of SUM on public dataset FT-Medium to ensure reproducibility: 1) **Mask Refinement:** This variant uses the refined masks produced by the mask-refinement module of SUM as pseudo labels for the unlabeled images in SA-250K, and is otherwise the same as the Vanilla baseline. 2) **SUM w/o Task Prompt:** SUM without task prompts but with uncertainty-aware loss and sampling. 3) **SUM Continuous TP**: This is a variant of SUM where the task prompt is utilized continuously in all rounds of interactive segmentation during training and inference (as opposed to just in the first round, as in SUM).

The results demonstrate the advantage of uncertainty-aware training over direct mask refinement. SUM w/o Task Prompt outperforms Mask Refinement, especially for part segmentation, which is the segmentation task that is not included in the human-annotated labels. Both models outperform Vanilla, indicating that mask refinement provides an advantage over a naive combination of unlabeled and labeled data. Additional experiments analyzing mask refinement are reported in Appendix D.

SUM is superior to SUM without Task Prompt across all segmentation tasks for a single-point prompt, but both models are essentially equivalent for 3 prompts. This suggests that task prompting is very useful when there is more ambiguity as to which mask is desired by the user. In fact, SUM Continuous TP underperforms SUM for 3-point prompts, suggesting that persistent task prompting may be counterproductive as semantic ambiguity decreases.

**Additional Ablation** In Appendix F.2, we compare SUM to a version of SUM without uncertainty-aware sampling. In Appendix Table 6, we provide a comprehensive comparison between SAM and SUM implemented with different backbones, including ViT-B, ViT-L, ViT-H. Our model consistently outperforms SAM across different model backbones for all mask types and interactive rounds. The improvement is most significant for single-point prompt segmentation of salient objects. In Appendix F.1, we compare SUM and different variants of SUM via different lightweight fine-tuning methods.

# 5 Discussion and Future Work

Fine-tuning foundation models effectively is an important challenge. Our work suggests that a key consideration is how to account for varying degrees of diversity and label quality in the available data. Specifically, we show that in the case of segmentation, incorporating a diverse unlabeled dataset can be effective in preserving the generalization ability of the original foundation model, as long as they are utilized in an uncertainty-aware manner.

As detailed in Appendix D, our methods have limitations: the quality of mask refinement in SUM is dependent on the initial performance of the SAM model. This suggests that leveraging SUM to improve pseudo-label quality in the training data could lead to further improvement. Our work suggests some possible avenues for future research to improve the training of foundation models. The uncertainty maps generated by SUM can potentially be used to improve pseudo-label filtering in foundation-model data engines. In fact, a human study reported in Appendix I found that filtering approaches utilized in SAM are unable to accurately discern pseudo labels with inaccurate boundaries. Finally, the efficacy of the proposed task prompts suggests that it may be possible to develop automatic label-generation schemes to produce pseudo-labels associated with more diverse tasks.

## Acknowledgments and Disclosure of Funding

The authors acknowledge Markus Woodson for valuable discussions and feedback. They also thank Weicheng Zhu and Yunzhen Feng for their assistance with manuscript proofreading. Kangning Liu was supported by NSF grant NRT-1922658. Krzysztof J. Geras was supported by The Shifrin Myers Breast Cancer Discovery Fund (SMBCDF) and NYU Langone Health, Department of Surgery.

## Footnotes

[1]The regions in the target mask not present in the predicted mask are taken as positive prompts, while the region in the predicted mask not present in the target mask are taken as negative prompts.

[2]These are the logits fed through a sigmoid and binarized to produce the SAM pseudo-labels.

## References

[1] Alexander Kirillov, Eric Mintun, Nikhila Ravi, Hanzi Mao, Chloe Rolland, Laura Gustafson, Tete Xiao, Spencer Whitehead, Alexander C Berg, Wan-Yen Lo, et al. Segment anything. *ICCV*, 2023.

[2] Dengsheng Zhang, Md Monirul Islam, and Guojun Lu. A review on automatic image annotation techniques. *Pattern Recognition*, 45(1):346–362, 2012.

[3] Venkatesh N Murthy, Subhransu Maji, and R Manmatha. Automatic image annotation using deep learning representations. In *Proceedings of the 5th ACM on International Conference on Multimedia Retrieval*, 2015.

[4] Lei Ke, Mingqiao Ye, Martin Danelljan, Yu-Wing Tai, Chi-Keung Tang, Fisher Yu, et al. Segment anything in high quality. *NeurIPS*, 2024.

[5] Tianrun Chen, Lanyun Zhu, Chaotao Ding, Runlong Cao, Shangzhan Zhang, Yan Wang, Zejian Li, Lingyun Sun, Papa Mao, and Ying Zang. Sam fails to segment anything?–sam-adapter: Adapting sam in underperformed scenes: Camouflage, shadow, and more. *arXiv preprint arXiv:2304.09148*, 2023.

[6] Jay N Paranjape, Nithin Gopalakrishnan Nair, Shameema Sikder, S Swaroop Vedula, and Vishal M Patel. Adaptivesam: Towards efficient tuning of sam for surgical scene segmentation. *arXiv preprint arXiv:2308.03726*, 2023.

[7] Shizhan Gong, Yuan Zhong, Wenao Ma, Jinpeng Li, Zhao Wang, Jingyang Zhang, Pheng-Ann Heng, and Qi Dou. 3dsam-adapter: Holistic adaptation of sam from 2d to 3d for promptable medical image segmentation. *arXiv preprint arXiv:2306.13465*, 2023.

[8] Yaqin Li, Dandan Wang, Cao Yuan, Hao Li, and Jing Hu. Enhancing agricultural image segmentation with an agricultural segment anything model adapter. *Sensors*, 23(18):7884, 2023.

[9] Vijay Badrinarayanan, Alex Kendall, and Roberto Cipolla. Segnet: A deep convolutional encoder-decoder architecture for image segmentation. *TPAMI*, 39(12):2481–2495, 2017.

[10] Liang-Chieh Chen, George Papandreou, Iasonas Kokkinos, Kevin Murphy, and Alan L Yuille. Deeplab: Semantic image segmentation with deep convolutional nets, atrous convolution, and fully connected crfs. *TPAMI*, 40(4):834–848, 2017.

[11] Sixiao Zheng, Jiachen Lu, Hengshuang Zhao, Xiatian Zhu, Zekun Luo, Yabiao Wang, Yanwei Fu, Jianfeng Feng, Tao Xiang, Philip HS Torr, et al. Rethinking semantic segmentation from a sequence-to-sequence perspective with transformers. In *CVPR*, 2021.

[12] Bowen Cheng, Ishan Misra, Alexander G Schwing, Alexander Kirillov, and Rohit Girdhar. Masked-attention mask transformer for universal image segmentation. In *CVPR*, 2022.

[13] Kaiming He, Georgia Gkioxari, Piotr Dollár, and Ross Girshick. Mask r-cnn. In *ICCV*, 2017.

[14] Xinlong Wang, Rufeng Zhang, Tao Kong, Lei Li, and Chunhua Shen. Solov2: Dynamic and fast instance segmentation. *NeurIPS*, 2020.

[15] Yanwei Li, Xinze Chen, Zheng Zhu, Lingxi Xie, Guan Huang, Dalong Du, and Xingang Wang. Attention-guided unified network for panoptic segmentation. In *CVPR*, 2019.

[16] Alexander Kirillov, Kaiming He, Ross Girshick, Carsten Rother, and Piotr Dollár. Panoptic segmentation. In *CVPR*, 2019.

[17] Konstantin Sofiiuk, Ilia Petrov, Olga Barinova, and Anton Konushin. f-brs: Rethinking back-propagating refinement for interactive segmentation. In *CVPR*, 2020.

[18] Yuying Hao, Yi Liu, Zewu Wu, Lin Han, Yizhou Chen, Guowei Chen, Lutao Chu, Shiyu Tang, Zhiliang Yu, Zeyu Chen, et al. Edgeflow: Achieving practical interactive segmentation with edge-guided flow. In *ICCV*, 2021.

[19] Xi Chen, Zhiyan Zhao, Feiwu Yu, Yilei Zhang, and Manni Duan. Conditional diffusion for interactive segmentation. In *ICCV*, 2021.

[20] Minghao Zhou, Hong Wang, Qian Zhao, Yuexiang Li, Yawen Huang, Deyu Meng, and Yefeng Zheng. Interactive segmentation as gaussion process classification. In *CVPR*, 2023.

[21] Jacob Devlin, Ming-Wei Chang, Kenton Lee, and Kristina Toutanova. Bert: Pre-training of deep bidirectional transformers for language understanding. *arXiv preprint arXiv:1810.04805*, 2018.

[22] Tom Brown, Benjamin Mann, Nick Ryder, Melanie Subbiah, Jared D Kaplan, Prafulla Dhariwal, Arvind Neelakantan, Pranav Shyam, Girish Sastry, Amanda Askell, et al. Language models are few-shot learners. *NeurIPS*, 2020.

[23] OpenAI. Gpt-4 technical report, 2023.

[24] Hugo Touvron, Thibaut Lavril, Gautier Izacard, Xavier Martinet, Marie-Anne Lachaux, Timo-thée Lacroix, Baptiste Rozière, Naman Goyal, Eric Hambro, Faisal Azhar, et al. Llama: Open and efficient foundation language models. *arXiv preprint arXiv:2302.13971*, 2023.

[25] Xinlong Wang, Wen Wang, Yue Cao, Chunhua Shen, and Tiejun Huang. Images speak in images: A generalist painter for in-context visual learning. In *CVPR*, 2023.

[26] Xinlong Wang, Xiaosong Zhang, Yue Cao, Wen Wang, Chunhua Shen, and Tiejun Huang. Seggpt: Segmenting everything in context. *arXiv preprint arXiv:2304.03284*, 2023.

[27] Xueyan Zou, Jianwei Yang, Hao Zhang, Feng Li, Linjie Li, Jianfeng Gao, and Yong Jae Lee. Segment everything everywhere all at once. *arXiv preprint arXiv:2304.06718*, 2023.

[28] Xu Zhao, Wenchao Ding, Yongqi An, Yinglong Du, Tao Yu, Min Li, Ming Tang, and Jinqiao Wang. Fast segment anything. *arXiv preprint arXiv:2306.12156*, 2023.

[29] Chaoning Zhang, Dongshen Han, Yu Qiao, Jung Uk Kim, Sung-Ho Bae, Seungkyu Lee, and Choong Seon Hong. Faster segment anything: Towards lightweight sam for mobile applications. *arXiv preprint arXiv:2306.14289*, 2023.

[30] Yang Liu, Muzhi Zhu, Hengtao Li, Hao Chen, Xinlong Wang, and Chunhua Shen. Matcher: Segment anything with one shot using all-purpose feature matching. *arXiv preprint arXiv:2305.13310*, 2023.

[31] Renrui Zhang, Zhengkai Jiang, Ziyu Guo, Shilin Yan, Junting Pan, Hao Dong, Peng Gao, and Hongsheng Li. Personalize segment anything model with one shot. *arXiv preprint arXiv:2305.03048*, 2023.

[32] Feng Li, Hao Zhang, Peize Sun, Xueyan Zou, Shilong Liu, Jianwei Yang, Chunyuan Li, Lei Zhang, and Jianfeng Gao. Semantic-sam: Segment and recognize anything at any granularity. *ECCV*, 2024.

[33] Haoxiang Wang, Pavan Kumar Anasosalu Vasu, Fartash Faghri, Raviteja Vemulapalli, Mehrdad Farajtabar, Sachin Mehta, Mohammad Rastegari, Oncel Tuzel, and Hadi Pouransari. Sam-clip: Merging vision foundation models towards semantic and spatial understanding. *arXiv preprint arXiv:2310.15308*, 2023.

[34] Junde Wu, Rao Fu, Huihui Fang, Yuanpei Liu, Zhaowei Wang, Yanwu Xu, Yueming Jin, and Tal Arbel. Medical sam adapter: Adapting segment anything model for medical image segmentation. *arXiv preprint arXiv:2304.12620*, 2023.

[35] Kaidong Zhang and Dong Liu. Customized segment anything model for medical image segmentation, 2023.

[36] Xinyu Xiong, Churan Wang, Wenxue Li, and Guanbin Li. Mammo-sam: Adapting foundation segment anything model for automatic breast mass segmentation in whole mammograms. In *International Workshop on Machine Learning in Medical Imaging*, 2023.

[37] Can Cui, Ruining Deng, Quan Liu, Tianyuan Yao, Shunxing Bao, Lucas W Remedios, Yucheng Tang, and Yuankai Huo. All-in-sam: from weak annotation to pixel-wise nuclei segmentation with prompt-based finetuning. *arXiv preprint arXiv:2307.00290*, 2023.

[38] Dongjie Cheng, Ziyuan Qin, Zekun Jiang, Shaoting Zhang, Qicheng Lao, and Kang Li. Sam on medical images: A comprehensive study on three prompt modes. *arXiv preprint arXiv:2305.00035*, 2023.

[39] Deepa Anand, Gurunath Reddy M, Vanika Singhal, Dattesh D. Shanbhag, Shriram KS, Uday Patil, Chitresh Bhushan, Kavitha Manickam, Dawei Gui, Rakesh Mullick, Avinash Gopal, Parminder Bhatia, and Taha Kass-Hout. One-shot localization and segmentation of medical images with foundation models, 2023.

[40] Bertrand Chauveau and Pierre Merville. Segment anything by meta as a foundation model for image segmentation: a new era for histopathological images. *Pathology*, 2023.

[41] SeungKyu Kim, Hyun-Jic Oh, Seonghui Min, and Won-Ki Jeong. Evaluation and improvement of segment anything model for interactive histopathology image segmentation, 2023.

[42] Lucas Prado Osco, Qiusheng Wu, Eduardo Lopes de Lemos, Wesley Nunes Gonçalves, Ana Paula Marques Ramos, Jonathan Li, and José Marcato Junior. The segment anything model (sam) for remote sensing applications: From zero to one shot. *International Journal of Applied Earth Observation and Geoinformation*, 124:103540, 2023.

[43] Yonghui Wang, Wengang Zhou, Yunyao Mao, and Houqiang Li. Detect any shadow: Segment anything for video shadow detection, 2023.

[44] Xiao Feng Zhang, Tian Yi Song, and Jia Wei Yao. Deshadow-anything: When segment anything model meets zero-shot shadow removal, 2023.

[45] Leiping Jie and Hui Zhang. Adaptershadow: Adapting segment anything model for shadow detection, 2023.

[46] Shichang Liu, Junxin Chen, Ben-Guo He, Tao Chen, Gwanggil Jeon, and Wei Wang. Adapting segment anything model for shield tunnel water leakage segmentation. In *Workshop on Advanced Multimedia Computing for Smart Manufacturing and Engineering*, pages 13–18, 2023.

[47] Hongyi Zhang, Moustapha Cisse, Yann N Dauphin, and David Lopez-Paz. mixup: Beyond empirical risk minimization. *arXiv preprint arXiv:1710.09412*, 2017.

[48] Jaemin Na, Jung-Woo Ha, Hyung Jin Chang, Dongyoon Han, and Wonjun Hwang. Switching temporary teachers for semi-supervised semantic segmentation. *NeurIPS*, 2024.

[49] Zhen Zhao, Lihe Yang, Sifan Long, Jimin Pi, Luping Zhou, and Jingdong Wang. Augmentation matters: A simple-yet-effective approach to semi-supervised semantic segmentation. In *CVPR*, 2023.

[50] Hanzhe Hu, Fangyun Wei, Han Hu, Qiwei Ye, Jinshi Cui, and Liwei Wang. Semi-supervised semantic segmentation via adaptive equalization learning. *NeurIPS*, 2021.

[51] Yuchao Wang, Haochen Wang, Yujun Shen, Jingjing Fei, Wei Li, Guoqiang Jin, Liwei Wu, Rui Zhao, and Xinyi Le. Semi-supervised semantic segmentation using unreliable pseudo-labels. In *CVPR*, 2022.

[52] Yuyuan Liu, Yu Tian, Yuanhong Chen, Fengbei Liu, Vasileios Belagiannis, and Gustavo Carneiro. Perturbed and strict mean teachers for semi-supervised semantic segmentation. In *CVPR*, 2022.

[53] Yen-Cheng Liu, Chih-Yao Ma, Zijian He, Chia-Wen Kuo, Kan Chen, Peizhao Zhang, Bichen Wu, Zsolt Kira, and Peter Vajda. Unbiased teacher for semi-supervised object detection. *arXiv preprint arXiv:2102.09480*, 2021.

[54] Zhanghan Ke, Di Qiu, Kaican Li, Qiong Yan, and Rynson WH Lau. Guided collaborative training for pixel-wise semi-supervised learning. In *ECCV*, 2020.

[55] Xiaokang Chen, Yuhui Yuan, Gang Zeng, and Jingdong Wang. Semi-supervised semantic segmentation with cross pseudo supervision. In *CVPR*, 2021.

[56] Ying Jin, Jiaqi Wang, and Dahua Lin. Semi-supervised semantic segmentation via gentle teaching assistant. *NeurIPS*, 2022.

[57] Yuxi Wang, Junran Peng, and ZhaoXiang Zhang. Uncertainty-aware pseudo label refinery for domain adaptive semantic segmentation. In *ICCV*, 2021.

[58] Zhedong Zheng and Yi Yang. Rectifying pseudo label learning via uncertainty estimation for domain adaptive semantic segmentation. *IJCV*, 129(4):1106–1120, 2021.

[59] Nikita Araslanov and Stefan Roth. Self-supervised augmentation consistency for adapting semantic segmentation. In *CVPR*, 2021.

[60] Sheng Liu, Kangning Liu, Weicheng Zhu, Yiqiu Shen, and Carlos Fernandez-Granda. Adaptive early-learning correction for segmentation from noisy annotations. In *CVPR*, 2022.

[61] Tsung-Yi Lin, Priya Goyal, Ross Girshick, Kaiming He, and Piotr Dollár. Focal loss for dense object detection. In *ICCV*, 2017.

[62] Fausto Milletari, Nassir Navab, and Seyed-Ahmad Ahmadi. V-net: Fully convolutional neural networks for volumetric medical image segmentation. In *3DV*, 2016.

[63] Lu Qi, Jason Kuen, Weidong Guo, Tiancheng Shen, Jiuxiang Gu, Wenbo Li, Jiaya Jia, Zhe Lin, and Ming-Hsuan Yang. Fine-grained entity segmentation. *arXiv preprint arXiv:2211.05776*, 2022.

[64] Jun Hao Liew, Scott Cohen, Brian Price, Long Mai, and Jiashi Feng. Deep interactive thin object selection. In *WACV*, 2021.

[65] Xuebin Qin, Hang Dai, Xiaobin Hu, Deng-Ping Fan, Ling Shao, and Luc Van Gool. Highly accurate dichotomous image segmentation. In *ECCV*, 2022.

[66] Yi Zeng, Pingping Zhang, Jianming Zhang, Zhe Lin, and Huchuan Lu. Towards high-resolution salient object detection. In *ICCV*, 2019.

[67] Ho Kei Cheng, Jihoon Chung, Yu-Wing Tai, and Chi-Keung Tang. Cascadepsp: Toward class-agnostic and very high-resolution segmentation via global and local refinement. In *CVPR*, 2020.

[68] Menglin Jia, Mengyun Shi, Mikhail Sirotenko, Yin Cui, Claire Cardie, Bharath Hariharan, Hartwig Adam, and Serge Belongie. Fashionpedia: Ontology, segmentation, and an attribute localization dataset. In *ECCV*, 2020.

[69] Jianshu Li, Jian Zhao, Yunchao Wei, Congyan Lang, Yidong Li, Terence Sim, Shuicheng Yan, and Jiashi Feng. Multiple-human parsing in the wild. *arXiv preprint arXiv:1705.07206*, 2017.

[70] Alexander Kapitanov. EasyPortrait Dataset. `https://www.kaggle.com/datasets/kapitanov/easyportrait`, 2023. Accessed: 2023-09-01.

[71] Vignesh Ramanathan, Anmol Kalia, Vladan Petrovic, Yi Wen, Baixue Zheng, Baishan Guo, Rui Wang, Aaron Marquez, Rama Kovvuri, Abhishek Kadian, et al. Paco: Parts and attributes of common objects. In *CVPR*, 2023.

[72] Taojiannan Yang, Yi Zhu, Yusheng Xie, Aston Zhang, Chen Chen, and Mu Li. Aim: Adapting image models for efficient video action recognition. *arXiv preprint arXiv:2302.03024*, 2023.

[73] Edward J Hu, Phillip Wallis, Zeyuan Allen-Zhu, Yuanzhi Li, Shean Wang, Lu Wang, Weizhu Chen, et al. Lora: Low-rank adaptation of large language models. In *ICLR*, 2021.

[74] Ilya Loshchilov and Frank Hutter. Decoupled weight decay regularization. *arXiv preprint arXiv:1711.05101*, 2017.

[75] Yanfei Song, Bangzheng Pu, Peng Wang, Hongxu Jiang, Dong Dong, Yongxiang Cao, and Yiqing Shen. Sam-lightening: A lightweight segment anything model with dilated flash attention to achieve 30 times acceleration, 2024.

[76] Ning Xu, Brian Price, Scott Cohen, Jimei Yang, and Thomas S Huang. Deep interactive object selection. In *CVPR*, 2016.

[77] Xiang Li, Tianhan Wei, Yau Pun Chen, Yu-Wing Tai, and Chi-Keung Tang. Fss-1000: A 1000-class dataset for few-shot segmentation. In *CVPR*, 2020.

[78] Jianping Shi, Qiong Yan, Li Xu, and Jiaya Jia. Hierarchical image saliency detection on extended cssd. *TPAMI*, 38(4):717–729, 2015.

[79] Ming-Ming Cheng, Niloy J Mitra, Xiaolei Huang, Philip HS Torr, and Shi-Min Hu. Global contrast based salient region detection. *TPAMI*, 37(3):569–582, 2014.

[80] Chuan Yang, Lihe Zhang, Huchuan Lu, Xiang Ruan, and Ming-Hsuan Yang. Saliency detection via graph-based manifold ranking. In *CVPR*, 2013.

[81] Lu Qi, Jason Kuen, Yi Wang, Jiuxiang Gu, Hengshuang Zhao, Philip Torr, Zhe Lin, and Jiaya Jia. Open world entity segmentation. *TPAMI*, 2022.

[82] Stephan R Richter, Vibhav Vineet, Stefan Roth, and Vladlen Koltun. Playing for data: Ground truth from computer games. In *ECCV*, 2016.

[83] Marius Cordts, Mohamed Omran, Sebastian Ramos, Timo Rehfeld, Markus Enzweiler, Rodrigo Benenson, Uwe Franke, Stefan Roth, and Bernt Schiele. The cityscapes dataset for semantic urban scene understanding. In *CVPR*, 2016.

[84] Lingzhi Zhang, Shenghao Zhou, Simon Stent, and Jianbo Shi. Fine-grained egocentric hand-object segmentation: Dataset, model, and applications. In *ECCV*, 2022.

[85] Nadav Cohen, Yael Newman, and Ariel Shamir. Semantic Segmentation in Art Paintings. *Computer Graphics Forum*, 2022.

[86] Lei Yang, Yan Zi Wei, Yisheng He, Wei Sun, Zhenhang Huang, Haibin Huang, and Haoqiang Fan. ishape: A first step towards irregular shape instance segmentation. *arXiv preprint arXiv:2109.15068*, 2021.

[87] Cameron Trotter, Georgia Atkinson, Matt Sharpe, Kirsten Richardson, A Stephen McGough, Nick Wright, Ben Burville, and Per Berggren. Ndd20: A large-scale few-shot dolphin dataset for coarse and fine-grained categorisation. *arXiv preprint arXiv:2005.13359*, 2020.

[88] Jungseok Hong, Michael Fulton, and Junaed Sattar. Trashcan: A semantically-segmented dataset towards visual detection of marine debris. *arXiv preprint arXiv:2007.08097*, 2020.

[89] Jiazhou Chen, Yanghui Xu, Shufang Lu, Ronghua Liang, and Liangliang Nan. 3-d instance segmentation of mvs buildings. *IEEE Transactions on Geoscience and Remote Sensing*, 60:1–14, 2022.

# Supplementary Material: Uncertainty-aware Fine-tuning of Segmentation Foundation Models

## Contents

# A  Checklist Details

## A.1  License of the Assets

**Licence for the Codes**   We build upon SAM [1], which is under Apache-2.0 license `https://www.apache.org/licenses/LICENSE-2.0`

We reproduce the code for AugSeg [49], U2PL [51], Adapter [72], LoRA [73, 35], all of which are under MIT License according to `https://opensource.org/licenses/MIT`. HQ-SAM [4] is under Apache-2.0 license `https://www.apache.org/licenses/LICENSE-2.0`

**Licence for the Dataset**   HQ-Seg-44k [4] is under Apache-2.0 license `https://www.apache.org/licenses/LICENSE-2.0`.

COIFT [64] is under Attribution-NonCommercial 4.0 International`https://creativecommons.org/licenses/by-nc/4.0/deed.en`.

DIS [65] is under Apache-2.0 license `https://www.apache.org/licenses/LICENSE-2.0`.

ThinObject5K [65] is under Apache-2.0 license `https://www.apache.org/licenses/LICENSE-2.0`.

HR-SOD [66] is under MIT License according to `https://opensource.org/licenses/MIT`.

VOCEVAL [67] and BIG [67] are under MIT License according to `https://opensource.org/licenses/MIT`.

EntitySeg [63] is under CC BY-NC 4.0 license `https://creativecommons.org/licenses/by-nc/4.0/`.

Paco [71] is under MIT License according to `https://opensource.org/licenses/MIT`.

Fashionpedia [68] is under Creative Commons Attribution 4.0 License `https://creativecommons.org/licenses/by/4.0/legalcode`.

Easyportrait [70] is under Creative Commons Attribution-ShareAlike 4.0 International License `http://creativecommons.org/licenses/by-sa/4.0/`.

The Multi-Human Parsing [69] dataset is made freely available to academic and non-academic entities for non-commercial purposes such as academic research, teaching, scientific publications, or personal experimentation. The license terms is provided at `https://lv-mhp.github.io/`.

## A.2  Broader Impacts

We want to comment on potential societal outcomes associated with this work. The training of large-scale foundation models incurs carbon emissions, which contributes to global warming and raises concerns about environmental sustainability within the AI research community. Additionally, the black-box characteristics of these deep-learning models may result in unintended biases in practical applications. Nonetheless, we hope these effects may be offset to some extent when the models are applied in domains such as medical image segmentation. We believe that advancements in accurate segmentation foundation models have positive societal implications across a variety of fields, including medicine, autonomous driving, and creative content generation.

# B  Implementation Details

## B.1  Training

In this section, we present comprehensive training details for our proposed Segmentation with Uncertainty Model (SUM), including loss functions, optimization strategy, interactive-segmentation training pipeline, and the selection and use of human-labeled and unlabeled data.

**Training Resources**   The original SAM [1] model was trained using 256 A100 GPUs. For fine-tuning, we default to 40 Nvidia A100 80GB GPUs for fine-tuning, contingent on the availability of

these GPUs. Regarding the SUM model deployed in the HQ-SAM framework [4], we adhere to the HQ-SAM configuration, utilizing 8 GPUs to ensure a fair comparison.

**Losses**　For the human-annotated data, SUM minimizes the same training loss as SAM: a linear combination of focal loss [61] and Dice loss [62] with the same ratio of 20:1. For the unlabeled data, we propose to instead utilize uncertainty-aware versions of both losses, which are then combined in the same way. The hyperparameter $\theta$ in the uncertainty-aware Dice loss is set to 0.6. We study the effect of varying this hyperparameter in Appendix F.5.

Following SAM, we train the IoU prediction head by optimizing the mean-square-error between the predicted IoU and the IoU of the predicted mask compared to the target mask. This is then incorporated into the mask loss, applying a scaling factor of 1.0.

**Training Procedure**　Following SAM [1], we use the AdamW [74] optimizer. The learning rate is set at $1 \times 10^{-5}$. Our fine-tuning process takes 6,000 iterations, which is equivalent to a single SA-250K epoch, with a batch size of 40 images. To account for GPU memory constraints, we train with a single image and up to 30 randomly sampled masks per GPU. Consistent with the SAM framework [1], we apply a layer-wise learning rate decay of 0.8. We do not apply data-augmentation techniques. SUM is initialized from an SAM ViT-H model. Training is distributed across 40 GPUs.

**Interactive Segmentation Training**　We conduct interactive training for 7 rounds. The initial round randomly selects between a bounding box or a point prompt with equal probability. This is followed by a sequence of five rounds, where point prompts are sampled as follows. 'Positive' prompts are selected from regions within the target mask that are absent in the predicted mask, while 'negative' prompts are selected from areas included in the predicted mask but not in the target mask. In line with the SAM framework, we incorporate an additional round without additional prompts after the final stage.

**Utilization of Human-labeled and Unlabeled Data**　We adjust the data sampling strategy to form the training batches, according to the relative frequency of human-annotated data and unlabeled data. For FT-Small and FT-Medium we include one human-annotated image for every unlabeled image. In the case of FT-Large, where human-labeled data surpass unlabeled data, we include one unlabeled image for every two human-labeled images.

**Pseudo-label Generation**　In the **fully automatic stage** of SAM's data engine [1], the model is provided with prompts of a 32×32 regular grid of points. The model then predicts a set of masks for each point. The masks are then filtered to discard low-quality ones. Specifically, there are three filters in SAM:

- The masks are selected based on the model's predicted IoU score, setting a minimum threshold of 88.0.

- For maintaining mask stability, SAM evaluated pairs of binary masks derived from the same soft mask but thresholded at slightly different values. A mask was retained only if the IoU between the binary masks, created by thresholding the logits at 0 and then at -1 and +1, was 95.0 or higher.

- Lastly, to eliminate non-informative masks, SAM excluded any mask that occupied 95% or more of the total image area.

Although the filtering mechanisms succeed in eliminating a significant portion of low-quality masks, their effectiveness is constrained when it comes to further distinguishing the accuracy of mask boundaries among the remaining masks. This limitation is evidenced by our human study detailed in Appendix I.1, which assesses the boundary precision of the filtered masks.

## B.2　Training Complexity and Computational Overhead

While the SUM framework adds steps to obtain uncertainty maps, they remain manageable within practical constraints, as detailed below.

- **Training the Uncertainty Map Generation Module:** Training the uncertainty map generation module is relatively efficient on the human-labeled samples. It involves tuning a small number of parameters and can be completed within 4 hours using 8 A100 GPUs.

- **Generating Uncertainty Map for the Unlabeled Image:** The computational overhead introduced by the SUM framework is minimal. The process of generating pseudo labels and uncertainty maps for unlabeled data utilizes the existing SAM encoder, thereby sharing the primary computational workload. Beyond pseudo-label generation, the additional time required to produce refined masks using a lightweight decoder is negligible compared to the time taken by the image encoder. ViT-H image encoder in SAM has 632M parameters while the prompt-based decoder only has 3.87M parameters. As per the reference [75], on an RTX4090, the decoder adds only approximately $2\%$ more computational time compared to the encoder.

- **Fine-Tuning the SAM Model:** Once the uncertainty map is generated, the fine-tuning SAM model using the uncertainty map is similar to the standard training used in SAM.

- **Iterative Point Prompt Sampling:** This method is used in standard SAM training as well, and replacing the original uniform sampling with weighted (uncertainty-aware) sampling results in a negligible training burden. Sampling a point from a 1024x1024 candidate pool using both methods can be completed within 0.006-0.009 seconds (average of 1000 runs) on the CPU of a MacBook.

- **Task Prompt:** The task prompt, a learnable single vector, is combined via element-wise addition with the embeddings from the SAM image encoder and is used only in the first round of interactive segmentation. The element-wise addition of two tensors is relatively fast.

- **Uncertainty-aware Loss Computation:** This involves thresholding and a weighted loss computation, which requires a similar running time to the original loss.

- **Inference Phase:** Uncertainty maps are utilized only during training. Once our model is trained, inference operates similarly to the SAM model, without additional computational burden. This results in negligible computational overhead.

## B.3 Evaluation

**Interactive Segmentation Evaluation** Evaluation is carried out using six interactive segmentation rounds. The initial prompt may be either a bounding box or a point. Given that point-based prompts are standard for interactive segmentation evaluations (as utilized in both SAM and HQ-SAM) our analysis predominantly concentrates on them.

**Point-prompt Sampling Method during Evaluation** Our point-prompt sampling during inference follows the one in SAM [1]. The initial point is deterministically chosen as the one farthest from the object's boundary. Subsequent points are selected similarly, as the farthest from the boundary of the error region between the ground-truth segmentation mask and the model prediction. It was first introduced by Xu et al. [76] and has become the norm in literature for evaluating interactive segmentation. It ensures consistency by deterministically selecting the initial point to be farthest from the object boundary. Subsequent points are placed farthest from the error region's boundary, which varies depending on the discrepancies between the ground truth and prior model predictions. While the *initial point is consistent* across all models, *subsequent points are model-specific*, in light of distinct error patterns.

In order to generate more challenging prompts, in certain experiments we perform boundary sampling instead. We utilize morphological operations for the precise identification of boundary points within a specified proximity to the mask's edge. In our specific configuration, this distance is set to 10 pixels.

**Comparison to Single-output Model** Given the possibility of semantic ambiguity when only a limited number of prompts is provided, we generally opt for a fair comparison strategy: in cases where several output masks are generated, we select the one that most closely aligns with the provided ground truth. However, this approach is not applicable in comparisons involving HQ-SAM, which, by design, produces a deterministic single output. To maintain fairness and consistency when comparing with HQ-SAM, we only utilize a single output from all other methods (which is fixed beforehand),

even if they have the capability to produce multiple outputs. The single output is set equal to the estimate corresponding to the fourth output token, following HQ-SAM. We include additional comparisons between models using deterministic single outputs in Appendix G.5.

## C  Additional Dataset Information

### C.1  Training Sets

**HQ-Seg-44k**  HQ-Seg-44k [4] comprises 44,320 highly precise image mask annotations for the salient object segmentation task. It was built by merging 6 pre-existing datasets: DIS [65] (train set), ThinObject-5K [64] (train set), FSS [77], ECSSD [78], MSRA-10K [79], and DUT-OMRON [80].

**EntitySeg [63] Train Set**  This dataset is a large-scale and high-quality dataset for entity segmentation [81]. It encompasses a diverse array of image domains and entities, enriched with a vast collection of high-resolution images and superior-quality mask annotations. Additionally, It is designed to be open-world and not restricted to predefined classes.

**Internal Dataset**  We collected a high-quality human annotated internal dataset containing 252,798 images with salient-object masks, 60,798 images with entity masks, and 153,046 images focused on part segmentation, primarily for human parsing and portrait. Each annotation in this dataset has been meticulously carried out by human annotators. The salient-object masks encompass a wide range of categories, such as people, vehicles, street signs, animals, household items, kitchenware, food, furniture, electronics, and personal accessories. The entity masks include both foreground elements (like people, vehicles, animals) and background components (including sky, road, grass, water, clouds, and mountains). The part segmentation masks predominantly focus on body parts, various clothing items, and accessories.

### C.2  Evaluation Sets

- **Salient Object**: We utilize COIFT [64], the validation sets of DIS [65] and ThinObject5K [65], HR-SOD [66], VOCEVAL [67], and BIG [67]. These datasets contain object-centric images with salient objects and binary masks. The training sets in DIS-VD and ThinObject5K are included in the HQSeg-44K dataset. COIFT, HRSOD, VOCEVAL, and BIG are used for zero-shot evaluation.

- **Entity**: Three EntitySeg [63] validation sets with 454, 459 and 401 images. These sets feature complex scene images with multiple entities ranging from foreground instances to background regions (e.g. sky, road).

- **Part**: Fashionpedia [68] is a large-scale dataset that contains a comprehensive ontology of fashion elements. We include the main evaluation set (containing 1148 images) with part segmentation annotation and a subset with even more fine-grained subpart masks (containing 861 images). We also include 1000 images from each of the Paco, LVIS, Multi-Human Parsing, and Easyportrait datasets. Paco [71] is an extensive detection dataset that enriches the traditional scope of object detection by incorporating detailed annotations for object parts and attributes. The Multi-Human Parsing [69] dataset aims to partition a crowd scene image into semantically consistent regions belonging to the body parts or clothes items while differentiating different identities. Easyportrait [70] is a large-scale image dataset for portrait segmentation and face parsing.

## D  Mask-refinement Module

### D.1  Design

We built the mask refinement module to identify systematic biases in the SAM pseudo labels before SUM training. The module is trained by finetuning a separate SAM model with mask-prompt segmentation on the human-annotated dataset. We limit fine-tuning to the decoder and prompt encoder, keeping the image encoder fixed, to prevent overfitting. In the final architecture, we only require the fine-tuned decoder and the prompt encoder, as the encoder is the same as the one in SAM.

## D.2 Training

The training data for the mask-refinement module is obtained by generating pseudo masks by applying SAM to human-annotated datasets. This provides the prompt-target pairings necessary for our fine-tuning process. These pseudo masks are generated using a single point prompt from each ground truth mask, mirroring the conditions under which SAM pseudo labels are produced. To address semantic ambiguity, the mask prompt is set to be the SAM mask output that is most aligned with the human annotation. We exclude pairs where the IoU between the SAM-generated pseudo label and the ground truth is below a threshold (set to 0.7). Additionally, we apply the same sample filtering technique as in SAM during mask generation to ensure mask quality (see Appendix B in SAM [1]). Fine-tuning is conducted on the prompt-target pairs, as shown in Fig. 6. The mask-refinement module also receives as input a box prompt encompassing the pseudo label mask, and the point prompt used to generate the pseudo labels. We empirically found that including isolated box or point prompts is counterproductive, so they were excluded (see Fig. 7). prompt.

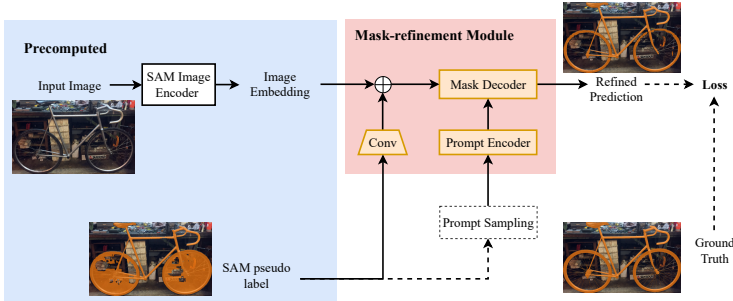

Figure 6: **Training of the mask-refinement module:** The mask-refinement module is implemented by fine-tuning the decoder and prompt encoder of SAM. By leveraging SAM's automatic mask generation, we acquire initial pseudo labels and corresponding image embeddings. Subsequently, these pseudo labels undergo a refinement process to produce the refined prediction. During training the prediction is fit using high-quality, human-annotated.

## D.3 Evaluation

The mask-refinement module refines SAM predictions via mask-prompt segmentation. Fig. 7 shows a comparison between SAM, our proposed mask-refinement module and three versions of HQ-SAM:

- **HQ-SAM** – Original HQ-SAM model that uses distorted ground truth as the mask prompt during training.

- **HQ-SAM MP** – Same as HQ-SAM but without incorporating separate box or point prompts during training.

- **HQ-SAM MP w SAM pseudo label** – Same as HQ-SAM MP but using SAM-generated pseudo labels as the mask prompt in place of the distorted ground truth, the approach taken by the original HQ-SAM.

For a fair comparison, all model variants were fine-tuned using only the HQSeg-44K [4] dataset, which contains human-annotated masks associated to the salient object segmentation task. The SAM pseudo labels are generated using using a single point prompt sampled from the ground truth segmentation mask. The proposed mask-refinement module clearly outperforms the alternatives, which overfit the salient-object segmentation task.

Fig. 8 presents an ablation analysis of the mask-refinement module, comparing the following ablated models fine-tuned using only the HQSeg-44K [4] :

- **Without Filtering** – Proposed mask-refinement module without pseudo-label filtering, which disregards pairs with an IoU below a set threshold.

- **With Synthetic Label** – Utilizes synthetic labels as mask prompt in training.

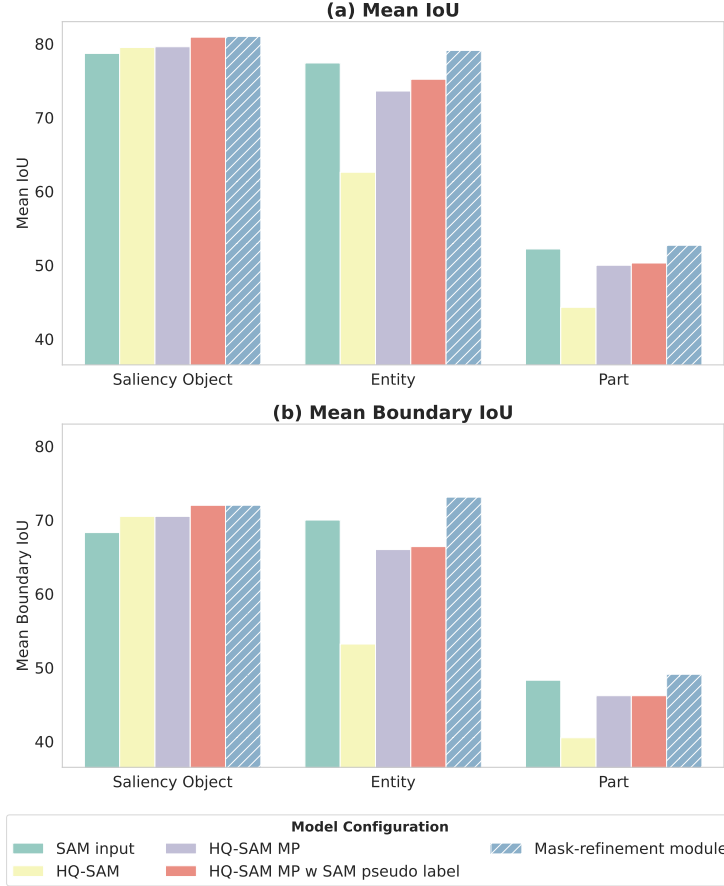

Figure 7: **Comparative evaluation of the mask-refinement module**. Comparison between SAM, our proposed mask-refinement module and three versions of HQ-SAM. The original HQ-SAM suffers from overfitting to specific tasks. HQ-SAM MP, which limits HQ-SAM to mask-prompt segmentation tasks while omitting individual box or point prompts during training achieves better generalization. Additional incorporation of SAM-generated pseudo labels yields improved results. Our mask-refinement module achieves the best performance.

- **With MAE Encoder** – Employs an encoder pretrained with Masked Autoencoder (MAE) weights.
- **With Encoder Adapter** – Incorporates a lightweight adapter into the image encoder.

The ablation results show that the proposed design clearly outperforms the ablated alternatives, particularly in terms of generalization to other segmentation tasks.

Table 3 presents the outcomes of utilizing the same mask-refinement module featured in Fig 7 and 8 to refine SAM predictions that were generated using a one-box prompt. The results demonstrate that our approach is also capable of refining pseudo labels generated by box prompts.

In Fig.9, we present a scatter plot that compares the boundary IoU of SAM predictions (produced using a single-point prompt) before and after refinement masks using the HRSOD dataset. For the majority of examples, mask-refinement module enhances the quality of the SAM prediction. However, in some cases the refined outputs are worse than the original prediction. Motivated in part by this, we propose leveraging the refined prediction via uncertainty-aware training instead. To evaluate the correlation between certainty (1-uncertainty) and SAM pseudo-label accuracy: we used the HRSOD dataset and set a threshold of 0.6 on the certainty, to identify regions high and low certainty areas. Results show that high-certainty regions have a superior accuracy (acc: $98\%$) over the uncertain ones (acc: $28.5\%$). When the threshold is 0.8, the accuracy in high-certainty regions increases to $99.2\%$, indicating a positive correlation between certainty and pseudo-label accuracy.

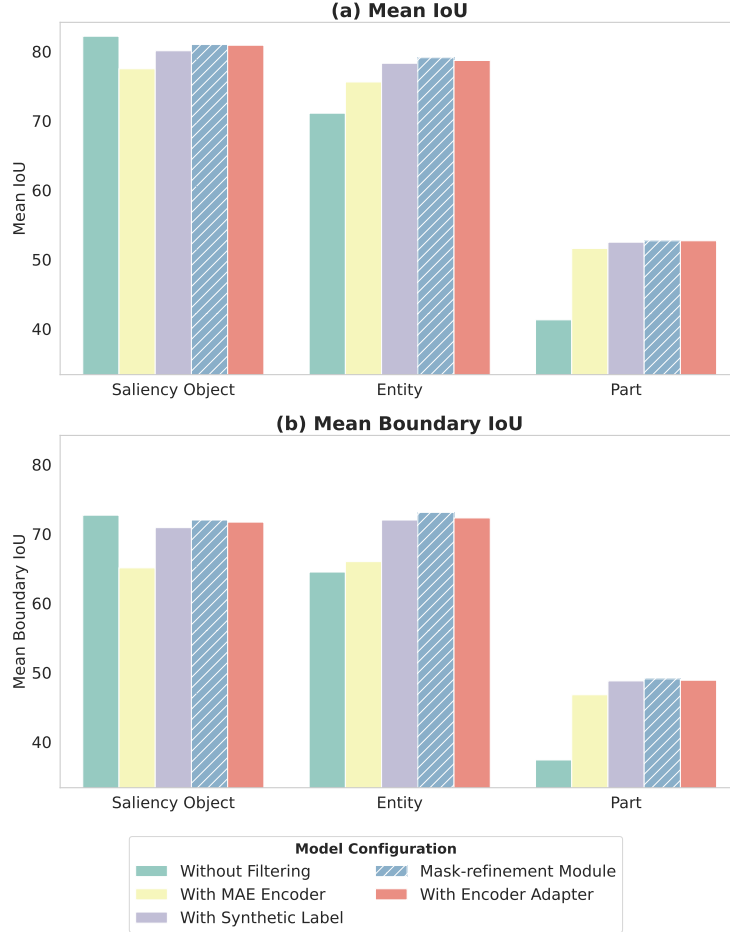

Figure 8: **Ablation analysis of the mask-refinement module.** Performance evaluation of different mask-refinement variants trained on the HQSeg-44K dataset. We use one-point prompt to generate the SAM predictions, which are then taken as the mask prompt input to all the model variants.

Table 3: **Mask refinement with box prompts**. Comparison of SAM and the result of applying the proposed mask-refinement module in Fig. 7 and 8, to SAM predictions generated by a one-box prompt. The refinement process yields enhanced results, with a notable improvement in boundary IoU metrics.

| Metrics | Salient Object | | Entity | | Part | |
| | SAM | Mask-refinement Module | SAM | Mask-refinement Module | SAM | Mask-refinement Module |
|---|---|---|---|---|---|---|
| mIoU | 84.5 | **86.0** | 82.6 | **82.7** | 66.6 | **67.0** |
| mBIoU | 75.4 | **77.8** | 76.4 | **77.4** | 63.2 | **63.6** |

We provide some visualization examples of the refinement results via the mask-refinement module in Fig. 10.

We further test whether our method improves upon the quality of SAM pseudo label generated via the automatic mask generation stage. Since there is no ground truth mask provided corresponding to the automatic mask generation, we conduct the human evaluation. The results are provided in Appendix I.2.

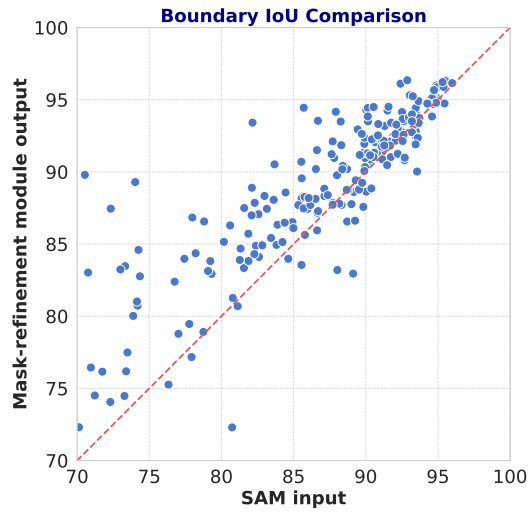

Figure 9: Comparison of IoU scores between one-point prompt SAM predictions and the corresponding refined prediction produced by mask-refinement module. Refinement improves segmentation quality for most examples.

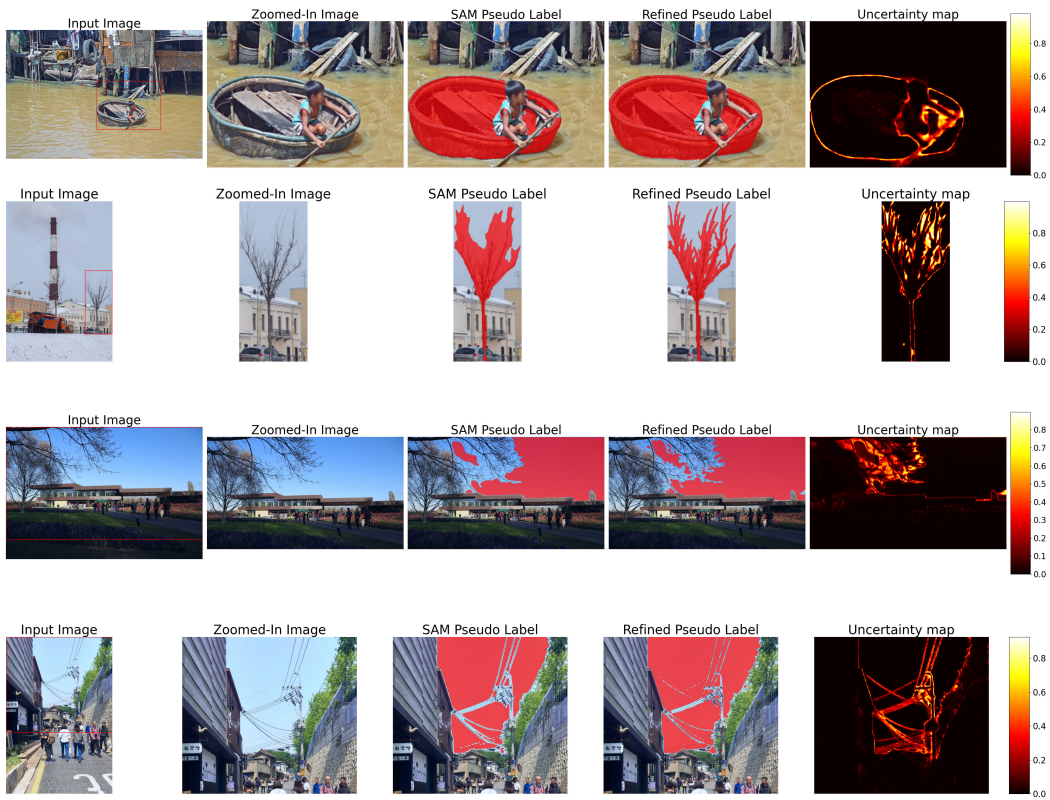

Figure 10: Visual examples of the refinement results via the mask-refinement module.

# E Semi-supervised Baseline

## E.1 Implementation Details

**AugSeg** AugSeg [49] employs a teacher-student framework. Following SAM [1], we use the AdamW [74] optimizer. The learning rate is set at $1 \times 10^{-5}$.

For the human-annotated data, we employ SAM's training procedure to train the student network, with both point and box prompts. We conduct interactive segmentation for 7 rounds. We minimize the same training loss as SAM: a linear combination of focal loss [61] and Dice loss [62] with the same ratio of 20:1.

For the unlabeled data, we apply a consistency loss based on predictions from two augmented versions of the same image, processed by the student and teacher networks. We binarize the prediction from the teacher network as the target, focal loss and Dice loss are added between the prediction from the teacher network and student network with a weight ratio of 20:1. We found that this works better than directly computing the KL divergence or mean square errors between two predictions. We follow the augmentation process in [49]. Interactive segmentation requires prompts sampled from target labels. We utilize model-generated pseudo-labels to generate the prompts for unlabeled data in two different ways:

- **AugSeg Mixed-Prompt**, which employs a combination of the point and box prompts. In scenarios where only a single prompt is employed, it may be either a point or a box prompt. Conversely, when multiple prompts are utilized, they may either be a mix of point and box prompts or exclusively point prompts;

- **AugSeg Box-Prompt**, which exclusively employs box prompts, applicable solely in single prompt scenarios.

In scenarios where a single prompt (either point or box) is provided, SAM is designed to yield three outputs to accommodate ambiguity associated with the target mask. To incorporate consistency loss between two sets of predictions, we explored two approaches:

- **AugSeg Mixed-Prompt** This is the default setting, where we select the most similar prediction to the pseudo-label from each set, and add consistency loss between these two predictions. This approach aligns with the SAM training protocol, where only the prediction most closely matches the target mask is considered for loss calculation.

- **AugSeg Mixed-Prompt(multi-consistency)** This variant computes the consistency loss between corresponding predictions from both sets, as determined by the same output token. Given that there are three output tokens (each for one output) for single-prompt scenarios, this results in three separate consistency losses.

In addition to sampling prompts from the pseudo-label, we employ a uniform prompt sampling strategy, denoted as **AugSeg Uniform Point-Prompt**. This approach is akin to the automatic mask generation process utilized in SAM. Specifically, we provide the model with a regular grid of points as prompts. For each point, the model predicts a set of masks potentially representing valid objects. We then compute the consistency loss between the corresponding predictions from both sets, as determined by the same output token. Such a non-discriminatory approach is feasible exclusively in single prompt scenarios, given the absence of target masks for interactive prompt sampling. However, employing this strategy has resulted in significantly inferior outcomes in our experiments. One plausible explanation could be the absence of filtering mechanisms similar to those in SAM's pseudo-label generation process, resulting in the retention of plenty low-quality predictions serving as regularization targets.

In the Table 4, we report the single-point prompt mean IoU comparison of different model variants fine-tuned on FT-Small. This shows that adding multiple consistency losses actually hurts the model performance.

For multi-prompt scenarios, we sample interactive prompts for both teacher and student networks based on SAM pseudo-labels.

Table 4: Single-point prompt mean IoU comparison of AugSeg Mixed-Prompt with a version that adds multiple consistency losses

| Task | AugSeg Mixed-Prompt | AugSeg Mixed-Prompt (multi-consistency) | AugSeg Box-Prompt | AugSeg Uniform Point-Prompt |
|---|---|---|---|---|
| Salient object | 82.3 | 81.4 | **83.2** | 82.5 |
| Entity | **75.9** | 74.6 | 73.9 | 71.6 |
| Part | **48.4** | 44.6 | 43.5 | 37.8 |
| Overall | **68.8** | 66.8 | 67.0 | 64.2 |

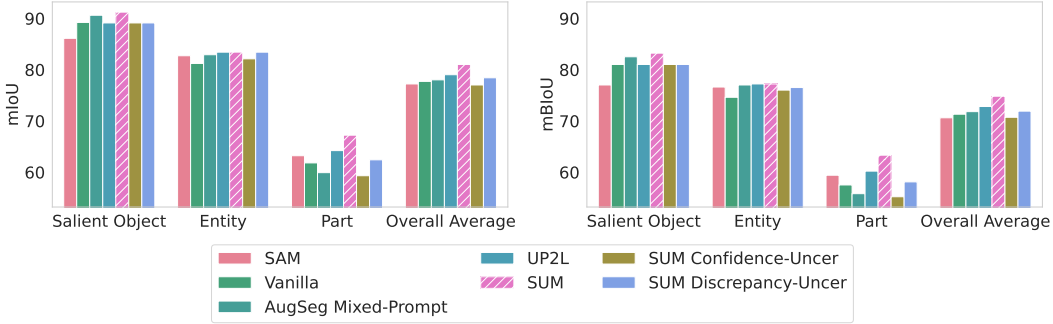

Figure 11: **Mean IoU and mean boundary IoU of 3 point-prompt segmentation evaluation.** Comparisons of models fine-tuned on FT-Small dataset with various strategies. SUM clearly outperforms all other strategies. Note that AugSeg Box-Prompt is not applicable as explained in Appendix E.

**U2PL**    Besides the standard loss function for both labeled and unlabeled examples, U2PL [51] also employs a contrastive loss to regularize the encoder's image embeddings based on the uncertainty of pseudo labels.

The original U2PL method relies on the confidence of model prediction logits for uncertainty quantification. However, our experiments show that this approach does not perform well with the segmentation foundation model. A possible explanation is that the foundation model, trained on a substantial volume of labeled and unlabeled data, begins with relatively high initial quality. This contrasts sharply with semi-supervised settings, where both data and labels are limited. To ensure a fair comparison, we adopt the same uncertainty map as used in SUM for the U2PL. Additionally, we incorporate task prompts into the U2PL framework. Following SAM [1], our training loss incorporates both pseudo and human labels, with an added contrastive loss as specified in the original paper.

Following SAM [1], we use the AdamW [74] optimizer. The learning rate is set at $1 \times 10^{-5}$.

For the human-annotated data, we employ SAM's training procedure to train the student network, with both point and box prompts. We conduct interactive segmentation for 7 rounds. We minimize the same training loss as SAM: a linear combination of focal loss [61] and Dice loss [62] with the same ratio of 20:1. In addition to the standard SAM loss, we also incorporate a contrastive loss in the embedding space for the pseudo-labeled data, we search the loss weight and find setting the contrastive loss weight to be 1 gets the best performance.

However, U2PL performance falls short of that achieved by SUM. This suggests that merely adding uncertainty to the image embedding space, while continuing to train the prediction space with noisy pseudo labels, is less effective. We additionally provide the three-point prompt comparison results plot corresponding to Table 1 in Fig 11. From this, we observe that AugSeg's performance is comparable to that of Vanilla, while SUM surpasses both in terms of these metrics.

**SUM Discrepancy-Uncertainty**    In this variant of the SUM model (SUM Discrepancy-Uncertainty), we adopt methodologies previously established for domain adaptive segmentation by Araslanov et al. [59] and Zheng et al. [58]. This involves generating two distinct predictions for a specified target mask and using the discrepancy between these predictions to quantify uncertainty.

We evaluated two different architectures: a conventional teacher-student framework, as described by Araslanov and Roth [59], and a dual-decoder structure based on the work of Zheng et al. [58]. In the latter, a single encoder feeds into two separate decoders that each produce a prediction. To clarify the target mask for prediction, we employed a bounding box derived from the pseudo label as a prompt. Our experimental results indicate that the teacher-student framework generally outperforms the dual-decoder setup. Nonetheless, neither approach matches the efficacy of our proposed method, which involves training a specialized mask refinement module utilizing high-quality human annotations.

### E.2 Discussion

**Semi-supervised Segmentation**    Directly integrating AugSeg within the interactive segmentation framework did not outperform the performance of directly utilizing SAM pseudo-labels as targets (i.e. the Vanilla baseline). We identify several potential avenues for enhancing semi-supervised methods in interactive segmentation scenarios:

- Designing better semi-supervised methods for interactive segmentation necessitates a better framework for prompt sampling. Generating prompts from model-generated pseudo-labels for unlabeled data is crucial. Here, the concept of uncertainty-aware prompt sampling, as introduced in SUM, shows potentials to improve model performance by avoiding producing misleading prompt locations.

- Moreover, the occasional inaccuracies in the output of the teacher network, which provides a guiding target mask for the student network's training, suggest that incorporating an uncertainty-aware component into the loss function could further refine performance.

- Our approach follows SAM, where consistency is enforced between the model's predictions and the pseudo-labels. The intuition is that the model should yield consistent outcomes in line with the pseudo-labels, despite variations in the sampling locations of the prompts from these pseudo-labels. This is orthogonal to augmentation-based semi-supervised strategies. Exploring the potential combination between the augmentation-based semi-supervised strategies and our proposed approach presents a compelling avenue for future research.

**Uncertainty-aware Segmentation**    Our approach differs fundamentally from previous approaches of uncertainty-aware segmentation in both 1) the generation of uncertainty maps and 2) the utilization of these uncertainty maps, as detailed below. We will clarify this in the revised manuscript.

The uncertainty-aware fine-tuning strategies generate uncertainty via model prediction logit values [51, 57], or use prediction differences from different heads to generate uncertainty [59, 58]. They are designed for domain-adaptive semantic segmentation or semi-supervised semantic segmentation, where labels or data are scarce, and segmentation quality is typically low. These methods are also specific to the limited task. In contrast, our proposed strategy is designed for the interactive binary segmentation foundation model, which has not been tackled by the previous method.

- **Uncertainty Map Generation:** Our main insight is to utilize external supervision to detect and correct systematic biases that accumulate during training. This design logic is fundamentally different from relying on the model self-training itself, as most previous models [51, 57]. SAM undergoes several rounds of self-training, leading to the accumulation and overfitting of errors in pseudo-labels. In our evaluation, the model frequently predicts erroneous regions with high confidence, and different heads concur on these incorrect areas due to the memorization of label noise [60]. Therefore, previous methods [51, 57, 59, 58] to generate uncertainty maps, do not capture this uncertainty effectively. To further improve segmentation quality, it is logical to integrate a form of scalable external supervision. This serves to identify and address the systematic biases that have accumulated during the training of foundation models. Our investigation into the design of mask-refinement models is an initial endeavor in this regard, and we are optimistic about future advancements in this field.

- **Utilization of the Uncertainty Map:** Our method is the first to introduce uncertainty-aware prompt sampling for training interactive segmentation models, representing a novel contribution to the field. In contrast, the semantic segmentation tasks addressed in [51, 57] do not involve prompt sampling. Additionally, we have tailored the uncertainty-aware focal and Dice loss to train the binary segmentation foundation model. They are not proposed in [51, 57].

Quantitative comparison with previous methods: As reported in Table 1, the proposed SUM method outperforms existing methods based on uncertainty quantification [57, 58], "SUM Confidence-Uncer" corresponds to [57], and "SUM Discrepancy-Uncer" corresponds to [58]. Table 1 also provides a comparison with UP2L [51], an uncertainty-aware semi-supervised segmentation method that uses uncertainty maps in the feature space via contrastive learning, which also performs worse than SUM (see also Figure 5).

**Continual Learning for Segmentation** We also want to comment on how our setting differs from the typical frameworks of continual learning in segmentation. Most of the papers on continual learning for segmentation are designed for class-incremental segmentation. However, in our case, it is not clear how to adopt such an idea for the binary mask segmentation case, where the aim is to not forget different segmentation tasks. Notably, several seminal studies within this domain focus on model weight distillation, creating a loss function that minimizes the discrepancy between predictions from previous and current models. This concept parallels the pseudo-labeling approach utilized in the SAM paper, which aims to maintain the prediction to be similar to pseudo-labels associated with different tasks.

Furthermore, some techniques strive to preserve previously learned knowledge by distilling intermediate feature maps. However, our experiments indicate that despite employing strategies like freezing the image encoder or using minimally light-weight fine-tuning methods such as HQ-SAM, task-specific knowledge degradation, or forgetting, still occurs in segmentation tasks.

# F Additional Ablation Study

## F.1 Ablation on Light-weight Fine-tuning

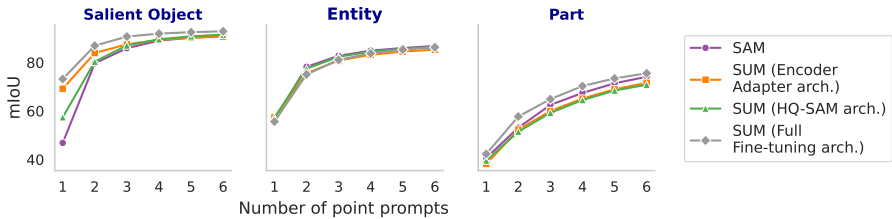

Figure 12: **Comparison of SAM and different versions of SUM fine-tuned on the FT-Small dataset.** Full parameter tuning shows the most significant improvement.

Fig. 12 compares SAM and different variants of SUM fine-tuned on FT-Small: (1) fine-tuning the decoder, the prompt encoder and the Adapter [72] plugged in the image encoder, (2) using the same lightweight fine-tuning scheme and architecture as HQ-SAM, (3) fine-tuning the whole SAM (the default setting). We observe that all variants of SUM outperform SAM on salient-object segmentation and retain the performance on entity and part segmentation. Besides, full fine-tuning clearly outperforms the other models.

All the ablation experiments are carried out on the public dataset FT-Medium to ensure reproducibility.

## F.2 Uncertainty-aware Sampling

Fig. 13 compares SUM to a version of SUM without uncertainty-aware sampling on two evaluation datasets (DIS-VD and ThinkObject5K val datasets), which contain objects with intricate structures. When evaluating, we provide prompts located close to the object boundaries, which are more challenging for the models. We observe that SUM with uncertainty-aware sampling has much better performance.

We compare SUM to a version of SUM without uncertainty-aware sampling during training for one-point segmentation in Table 5. When evaluating, we provide prompts located close to the object boundaries, which are more challenging for the models (see Appendix B.3). We observe that SUM fine-tuned with uncertainty-aware sampling has much better performance, especially for datasets with intricate structures like DIS-VD and ThinObject5k.

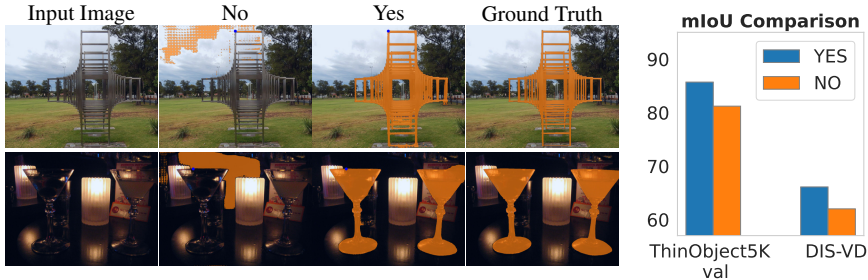

Figure 13: Comparison of SUM and a version of SUM without uncertainty-aware sampling on intricate structures when the prompts are close to object boundaries. In such cases, uncertainty-aware sampling provides visually superior results (left) and a higher single-prompt mean IoU comparison on two challenging test sets (right).

Table 5: Comparison of SUM and a version of SUM trained without uncertainty-aware sampling. Full table of **1-point-prompt mean IoU comparison** corresponding to Fig. 13. Here, we adopt a more challenging boundary-sensitive point selection tactics. Specifically, we select the "median" point from the set of boundary points, guaranteeing that the sample remains within the mask yet distanced from the geometric center.

| Metrics | Task | Dataset | w/o uncertainty sampling | SUM |
|---|---|---|---|---|
| mIoU | Salient object | COIFT | **91.7** | 91.0 |
| | | DIS-VD | 62.0 | **66.1** |
| | | HRSOD | **86.9** | 86.8 |
| | | hinObject5K | 81.2 | **85.7** |
| | | Big | 79.3 | **79.5** |
| | | Voceval | 77.5 | **78.0** |
| | | Average | 79.8 | **81.2** |
| | Entity | Entityseg 0 | 66.2 | **67.2** |
| | | Entityseg 1 | 68.6 | **69.0** |
| | | Entityseg 2 | 69.9 | **71.0** |
| | | Average | 68.2 | **69.1** |
| | Part | Bodypart | 50.9 | **54.4** |
| | | Easyportrait | **49.4** | 45.2 |
| | | Fashionpedia | 45.9 | **48.4** |
| | | Fashionpedia subpart | 30.0 | **31.6** |
| | | Paco | 43.3 | **43.4** |
| | | Average | 43.9 | **44.6** |
| | All | Average | 64.5 | **65.5** |
| mBIoU | Salient object | COIFT | **86.6** | 86.3 |
| | | DIS-VD | 56.0 | **59.1** |
| | | HRSOD | 79.5 | **80.0** |
| | | ThinObject5K | 70.6 | **75.5** |
| | | Big | 68.1 | **68.4** |
| | | Voceval | 63.7 | **64.8** |
| | | Average | 70.8 | **72.4** |
| | Entity | Entityseg 0 | 61.2 | **62.1** |
| | | Entityseg 1 | 63.7 | **64.0** |
| | | Entityseg 2 | 64.2 | **65.2** |
| | | Average | 63.0 | **63.8** |
| | Part | Bodypart | 46.6 | **49.5** |
| | | Easyportrait | **42.5** | 38.3 |
| | | Fashionpedia | 42.9 | **45.6** |
| | | Fashionpedia subpart | 27.9 | **29.7** |
| | | Paco | **41.9** | 41.8 |
| | | Average | 40.4 | **41.0** |
| | All | Average | 58.2 | **59.3** |

### F.3 Different Backbones

In Table 6, we provide a comprehensive comparison between SAM and SUM implemented with different backbones, including ViT-B, ViT-L, ViT-H. Our model consistently outperforms SAM across different model backbones for all segmentation tasks and interactive rounds. The improvement is most significant for one-point prompt segmentation of salient objects.

Table 6: We present a comparative evaluation of the mean Intersection over Union (IoU) and boundary IoU metrics for interactive segmentation, assessing the performance of SAM and SUM models across a range of Vision Transformer (ViT) backbones.

| Backbone | Metrics | Points | Salient Object | | Entity | | Part | |
|---|---|---|---|---|---|---|---|---|
| | | | SAM | SUM | SAM | SUM | SAM | SUM |
| ViT-B | mIoU | 1 | 70.9 | **78.4** | 73.2 | **74.7** | 52.3 | **55.0** |
| | | 6 | 90.9 | **92.5** | 86.8 | **88.2** | 76.1 | **77.8** |
| | mBIoU | 1 | 57.7 | **66.6** | 64.4 | **66.7** | 48.0 | **50.3** |
| | | 6 | 82.2 | **84.9** | 81.1 | **83.5** | 72.4 | **74.1** |
| ViT-L | mIoU | 1 | 78.0 | **84.5** | 77.0 | **77.9** | 52.7 | **52.9** |
| | | 6 | 91.3 | **93.3** | 86.9 | **88.7** | 74.6 | **76.0** |
| | mBIoU | 1 | 67.4 | **74.9** | 69.5 | **71.1** | 48.8 | **49.0** |
| | | 6 | 82.9 | **86.8** | 81.5 | **84.4** | 71.1 | **72.4** |
| ViT-H | mIoU | 1 | 78.7 | **85.2** | 77.4 | **79.8** | 52.2 | **55.3** |
| | | 6 | 91.2 | **93.5** | 86.8 | **89.2** | 74.5 | **77.1** |
| | mBIoU | 1 | 68.3 | **75.6** | 70.1 | **73.3** | 48.3 | **51.2** |
| | | 6 | 82.9 | **87.3** | 81.5 | **85.3** | 70.9 | **73.5** |

### F.4 Additional Details of Main Ablation Experiments

In Fig. 14 we provide an mIoU comparison of the Vanilla baseline and the ablated versions of SUM. We provide an mean Boundary IoU comparison of the Vanilla baseline and the ablated versions of SUM in Fig. 15. Detailed dataset-wise comparisons for interactive segmentation performance across different ablated versions are provided in Tables 7, 8 and 9.

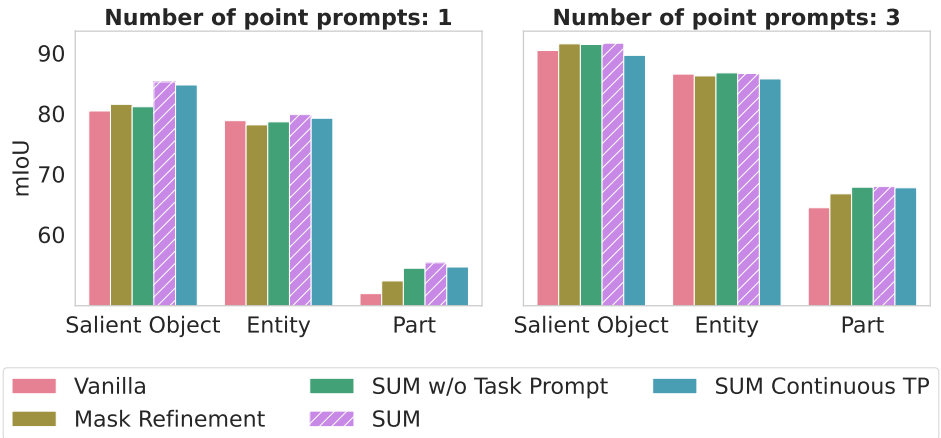

Figure 14: **Mean IoU for the ablation analysis**. This figure reports the interactive segmentation mean IoU of different ablated versions of SUM, demonstrating the individual gain provided by uncertainty-aware fine-tuning and task prompts.

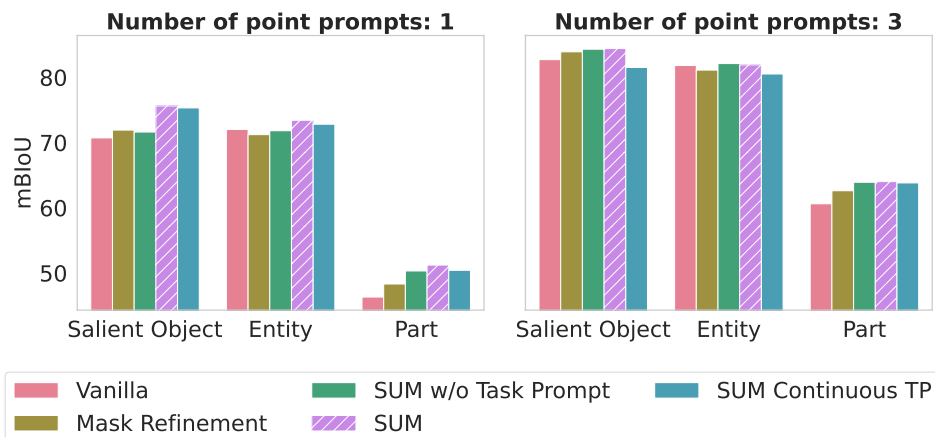

Figure 15: **Mean boundary IoU for Ablation study**. This figure reports the interactive segmentation mean boundary IoU of different ablated versions of SUM, demonstrating the individual gain provided by uncertainty-aware fine-tuning and task prompts.

Table 7: **Ablation study: one-point-prompt segmentation evaluation.** This table reports the results of different ablated versions of SUM, demonstrating the individual gain provided by uncertainty-aware fine-tuning and task prompts.

| Metrics | Task | Dataset | Vanilla | Mask-refinement | SUM w/o Task Prompt | SUM | SUM Continuous TP |
|---|---|---|---|---|---|---|---|
| mIoU | Salient object | COIFT | **94.7** | 94.5 | 93.9 | 93.1 | 94.5 |
| | | DIS-VD | 54.7 | 59.2 | 58.1 | **69.1** | 68.3 |
| | | HRSOD | 86.5 | 87.7 | 87.1 | **90.1** | 89.6 |
| | | ThinObject5K | 84.1 | 84.6 | 83.8 | 87.5 | **88.1** |
| | | Big | 79.5 | 80.2 | 80.7 | **86.6** | 83.0 |
| | | Voceval | 83.1 | 82.7 | 82.9 | **84.8** | 84.5 |
| | | Average | 80.4 | 81.5 | 81.1 | **85.2** | 84.7 |
| | Entity | Entityseg 0 | 77.1 | 76.3 | 77.1 | **78.1** | 77.4 |
| | | Entityseg 1 | 78.7 | 78.1 | 78.1 | **79.3** | 79.0 |
| | | Entityseg 2 | 80.6 | 80.0 | 80.7 | **81.9** | 81.2 |
| | | Average | 78.8 | 78.1 | 78.6 | **79.8** | 79.2 |
| | Part | Bodypart | 64.7 | 65.1 | **67.4** | 65.9 | 66.7 |
| | | Easyportrait | 45.7 | 52.3 | 53.2 | **55.3** | 53.8 |
| | | Fashionpedia | 53.3 | 55.3 | **58.4** | 58.4 | 57.2 |
| | | Fashionpedia subpart | 38.4 | 40.2 | 44.1 | **45.3** | 43.8 |
| | | Paco | 49.0 | 48.6 | 49.0 | **51.8** | 51.6 |
| | | Average | 50.2 | 52.3 | 54.4 | **55.3** | 54.6 |
| | All | Average | 69.3 | 70.3 | 71.0 | **73.4** | 72.8 |
| mBIoU | Salient object | COIFT | **90.2** | 89.7 | 88.8 | 88.4 | 90.1 |
| | | DIS-VD | 46.3 | 51.0 | 49.9 | **60.8** | 60.2 |
| | | HRSOD | 78.9 | 80.6 | 79.5 | **83.0** | 82.6 |
| | | ThinObject5K | 71.2 | 71.9 | 71.3 | 76.1 | **76.7** |
| | | Big | 67.6 | 69.2 | 70.0 | **75.2** | 71.3 |
| | | Voceval | 70.1 | 69.1 | 69.8 | 70.0 | **70.8** |
| | | Average | 70.7 | 71.9 | 71.6 | **75.6** | 75.3 |
| | Entity | Entityseg 0 | 70.5 | 69.6 | 70.7 | **71.8** | 71.3 |
| | | Entityseg 1 | 72.3 | 71.6 | 71.7 | **73.2** | 72.9 |
| | | Entityseg 2 | 73.0 | 72.4 | 73.1 | **75.0** | 74.4 |
| | | Average | 72.0 | 71.2 | 71.8 | **73.3** | 72.9 |
| | Part | Bodypart | 58.4 | 58.6 | **61.0** | 59.4 | 60.1 |
| | | Easyportrait | 38.0 | 44.2 | 45.1 | **47.4** | 45.7 |
| | | Fashionpedia | 51.3 | 53.2 | **55.9** | 55.9 | 54.8 |
| | | Fashionpedia subpart | 36.8 | 38.6 | 42.2 | **43.3** | 41.9 |
| | | Paco | 47.1 | 46.7 | 47.1 | **49.9** | 49.7 |
| | | Average | 46.3 | 48.3 | 50.3 | **51.2** | 50.4 |
| | All | Average | 62.3 | 63.3 | 64.0 | **66.4** | 65.9 |

Table 8: **Ablation study: three-point-prompt interactive segmentation evaluation.** This table reports the results of different ablated versions of SUM, demonstrating the individual gain provided by uncertainty-aware fine-tuning and task prompts.

| Metrics | Task | Dataset | Vanilla | Mask-refinement | SUM w/o Task Prompt | SUM | SUM Continuous TP |
|---|---|---|---|---|---|---|---|
| mIoU | Salient object | COIFT | 95.7 | **96.0** | 95.9 | 95.9 | 95.7 |
| | | DIS-VD | 75.9 | 79.0 | 79.3 | **80.1** | 75.2 |
| | | HRSOD | 93.0 | **93.6** | 93.2 | 93.3 | 92.5 |
| | | ThinObject5K | 91.6 | 92.3 | 92.1 | **92.4** | 90.8 |
| | | Big | 93.1 | **95.2** | 94.5 | 94.8 | 91.8 |
| | | Voceval | 92.9 | 92.9 | **93.3** | 93.2 | 91.7 |
| | | Average | 90.4 | 91.5 | 91.4 | **91.6** | 89.6 |
| | Entity | Entityseg 0 | 84.5 | 84.1 | **84.9** | **84.9** | 83.7 |
| | | Entityseg 1 | 86.5 | 86.4 | **86.6** | 86.5 | 85.7 |
| | | Entityseg 2 | 88.4 | 88.1 | **88.5** | 88.4 | 87.7 |
| | | Average | 86.5 | 86.2 | **86.7** | 86.6 | 85.7 |
| | Part | Bodypart | 74.3 | **76.9** | 76.4 | 74.9 | 76.6 |
| | | Easyportrait | 59.9 | 66.1 | 67.5 | **67.6** | 67.0 |
| | | Fashionpedia | 67.1 | 68.8 | **70.8** | **70.8** | 70.0 |
| | | Fashionpedia subpart | 58.8 | 59.5 | 61.4 | **62.7** | 61.8 |
| | | Paco | 62.0 | 62.2 | 62.9 | **63.5** | 63.2 |
| | | Average | 64.4 | 66.7 | 67.8 | **67.9** | 67.7 |
| | All | Average | 80.3 | 81.5 | 82.0 | **82.1** | 81.0 |
| mBIoU | Salient object | COIFT | 91.8 | **92.3** | 92.2 | 92.2 | 91.8 |
| | | DIS-VD | 68.6 | 71.8 | 72.6 | **73.5** | 67.3 |
| | | HRSOD | 87.9 | 88.5 | 88.6 | **88.7** | 87.0 |
| | | ThinObject5K | 82.1 | 83.4 | 83.3 | **83.9** | 80.9 |
| | | Big | 83.9 | 86.0 | **86.4** | 86.0 | 82.4 |
| | | Voceval | 81.9 | 81.3 | **82.7** | 82.1 | 79.6 |
| | | Average | 82.7 | 83.9 | 84.3 | **84.4** | 81.5 |
| | Entity | Entityseg 0 | 79.9 | 79.1 | **80.4** | 80.3 | 78.4 |
| | | Entityseg 1 | 82.0 | 81.6 | **82.2** | 82.0 | 80.9 |
| | | Entityseg 2 | 83.4 | 82.7 | **83.7** | 83.5 | 82.3 |
| | | Average | 81.8 | 81.1 | **82.1** | 81.9 | 80.5 |
| | Part | Bodypart | 68.9 | **71.1** | 70.7 | 69.4 | 70.8 |
| | | Easyportrait | 51.5 | 56.4 | 58.5 | **58.8** | 58.2 |
| | | Fashionpedia | 65.1 | 66.8 | **68.8** | **68.8** | 68.0 |
| | | Fashionpedia subpart | 57.1 | 58.0 | 59.9 | **61.0** | 60.1 |
| | | Paco | 60.5 | 60.6 | 61.4 | **62.0** | 61.7 |
| | | Average | 60.6 | 62.6 | 63.9 | **64.0** | 63.8 |
| | All | Average | 74.6 | 75.7 | 76.5 | **76.6** | 75.0 |

Table 9: **Ablation study: six-point-prompt interactive segmentation evaluation.** This table reports the results of different ablated versions of SUM, demonstrating the individual gain provided by uncertainty-aware fine-tuning and task prompts.

| Metrics | Task | Dataset | Vanilla | Mask-refinement | SUM w/o Task Prompt | SUM | SUM Continuous TP |
|---|---|---|---|---|---|---|---|
| mIoU | Salient object | COIFT | 96.0 | **96.2** | **96.2** | 96.1 | 95.9 |
| | | DIS-VD | 83.1 | 85.3 | **85.5** | 85.4 | 81.8 |
| | | HRSOD | 94.0 | **94.3** | 94.2 | **94.3** | 93.4 |
| | | ThinObject5K | 93.1 | **93.6** | 93.4 | 93.5 | 92.2 |
| | | Big | 96.5 | 96.8 | **97.0** | 96.9 | 95.6 |
| | | Voceval | **95.0** | **95.0** | 94.9 | 94.9 | 94.2 |
| | | Average | 92.9 | **93.6** | 93.5 | 93.5 | 92.2 |
| | Entity | Entityseg 0 | 87.2 | 87.1 | 87.4 | **87.5** | 86.5 |
| | | Entityseg 1 | 89.1 | 89.1 | **89.3** | 89.2 | 88.5 |
| | | Entityseg 2 | 90.8 | 90.7 | **90.9** | **90.9** | 90.2 |
| | | Average | 89.0 | 89.0 | **89.2** | **89.2** | 88.4 |
| | Part | Bodypart | 84.1 | 85.0 | **85.2** | 84.1 | 84.3 |
| | | Easyportrait | 75.4 | 78.7 | 78.9 | 78.5 | **79.0** |
| | | Fashionpedia | 77.1 | 77.6 | **78.0** | **78.0** | 77.4 |
| | | Fashionpedia subpart | 71.8 | 72.3 | 72.3 | **72.4** | 71.6 |
| | | Paco | 71.7 | 72.2 | **72.5** | 72.3 | 71.5 |
| | | Average | 76.0 | 77.2 | **77.4** | 77.1 | 76.8 |
| | All | Average | 86.0 | 86.7 | **86.8** | 86.7 | 85.9 |
| mBIoU | Salient object | COIFT | 92.4 | 92.7 | **92.8** | **92.8** | 92.2 |
| | | DIS-VD | 77.0 | 79.2 | **80.0** | 79.9 | 74.8 |
| | | HRSOD | 89.4 | 89.6 | 89.9 | **90.0** | 88.6 |
| | | ThinObject5K | 84.6 | 85.7 | 85.4 | **86.0** | 83.2 |
| | | Big | 88.9 | 89.2 | **90.2** | 90.0 | 87.2 |
| | | Voceval | 85.7 | 85.1 | **85.9** | 85.1 | 83.7 |
| | | Average | 86.3 | 86.9 | **87.4** | 87.3 | 84.9 |
| | Entity | Entityseg 0 | 83.1 | 82.9 | **83.5** | **83.5** | 82.0 |
| | | Entityseg 1 | 85.4 | 85.1 | **85.7** | 85.5 | 84.4 |
| | | Entityseg 2 | 86.6 | 86.3 | **86.9** | 86.8 | 85.7 |
| | | Average | 85.0 | 84.8 | **85.4** | 85.3 | 84.0 |
| | Part | Bodypart | 78.9 | 80.0 | **80.3** | 78.9 | 79.3 |
| | | Easyportrait | 67.3 | 70.6 | **71.0** | 70.1 | 70.8 |
| | | Fashionpedia | 75.3 | 75.9 | **76.4** | **76.4** | 75.6 |
| | | Fashionpedia subpart | 70.5 | **71.1** | **71.1** | **71.1** | 70.2 |
| | | Paco | 70.4 | 70.8 | **71.3** | 71.0 | 70.2 |
| | | Average | 72.5 | 73.7 | **74.0** | 73.5 | 73.2 |
| | All | Average | 81.1 | 81.7 | **82.2** | 81.9 | 80.6 |

## F.5 Threshold for the Uncertainty-aware Dice Loss

In Fig. 16, we present an ablation study of the uncertainty-aware Dice loss threshold, denoted by $\theta$. A smaller $\theta$ implies greater reliance on the pseudo labels. Our findings indicate that overly low values of $\theta$ adversely affect the model's performance in salient object and entity segmentation due to the effects of pseudo label noise. On the contrary, excessively high $\theta$ values may prevent the model from effectively leveraging part segmentation masks from the pseudo labels.

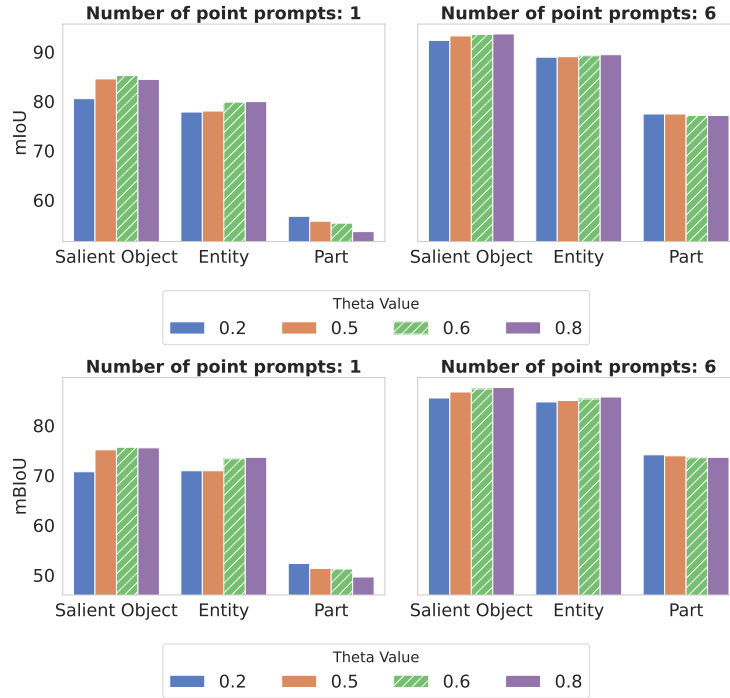

Figure 16: **Ablation study for the $\theta$ threshold in the uncertainty-aware Dice loss.** The figures compare the model's mean IoU and boundary IoU for various segmentation tasks and different $\theta$ values.

# G    Additional Quantitative Results

## G.1    Performance of SUM under Various Annotation Budgets

In this section we evaluate our proposed model (SUM) when fine-tuned under various annotation budgets, utilizing the FT-Medium and FT-Large datasets. We focus on point prompts and mean IoU. Additional results for box-prompt and mean Boundary IoU are provided in Appendix G.2. More granular dataset-wise results are reported in Appendix G.4.

***FT-Medium***    Fig. 17 shows a comparison between the Human-Annotation-Only baseline fine-tuned on HQ-Seg-44k and EntitySeg train set, SUM fine-tuned on FT-Medium (including SA-250K) and SAM. The Human-Annotation-Only baseline and SUM outperform SAM on the segmentation tasks associated with the human-annotated data (salient-object and entity segmentation). Although the Human-Annotation-Only baseline suffers a dramatic decrease in performance on part segmentation, the performance of SUM on part segmentation actually improves with respect to SAM by 3 points in mean IoU. This is notable because part segmentation is not explicitly represented in the human annotations, demonstrating SUM's capability to **enhance accuracy across general tasks**.

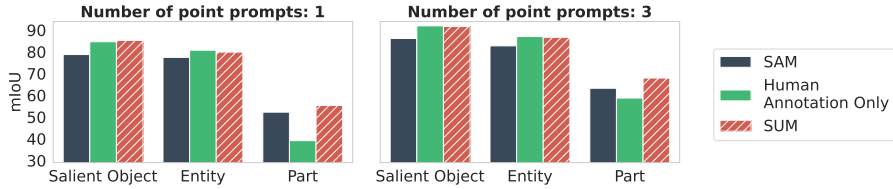

Figure 17: Interactive segmentation mean IoU of SAM, SUM fine-tuned on the FT-Medium dataset, and a Human-Annotation-Only baseline fine-tuned on a combination of the HQ-seg-44k and EntitySeg training sets. Fine-tuning only using human annotations improves performance for the tasks associated with the annotations (salient-object and entity segmentation), but deteriorates it dramatically for part segmentation.

***FT-Large***    Fig. 18 compares the Human-Annotation-Only baseline fine-tuned on the large-scale human-annotated internal train set, SUM, and the Vanilla baseline fine-tuned on FT-Large (including SA-250K). Human-Annotation-Only and SUM outperform Vanilla on salient-object segmentation. The three models perform similarly on entity segmentation. SUM is clearly superior in part segmentation. This shows the advantages of leveraging unlabeled data via SUM, even when the available human-annotated dataset is large-scale and diverse.

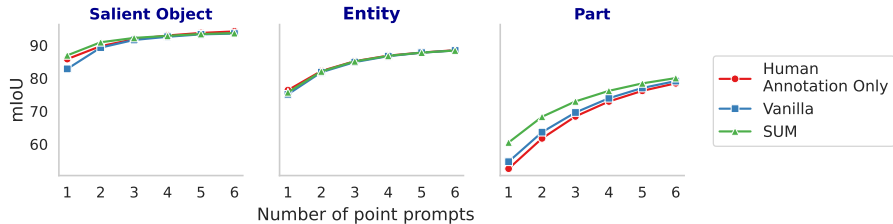

Figure 18: Comparison of the interactive segmentation mean IoU of SUM and a Vanilla baseline fine-tuned on the FT-Large dataset, and a Human-Annotation-Only baseline fine-tuned on a large-scale internal dataset. SUM outperforms the other models for part segmentation and matches the Human-Annotation-Only on the other tasks.

## G.2    Box Prompt Evaluation and Complementary Metrics

In this section we report evaluations for box prompts and complementary evaluation metrics (mean IoU or mean boundary IoU) to the figures in the main paper.

### G.2.1    Interactive Segmentation Evaluation with Initial Box Prompt

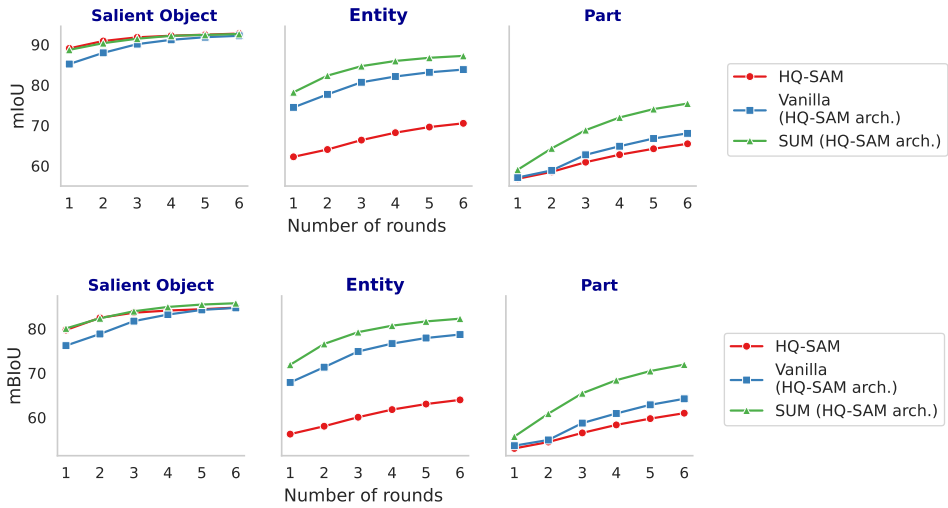

Figure 19: **Interactive segmentation evaluation with initial box prompt.** These results complement those in the top row of Fig. 4. It compares the performance of HQ-SAM (fine-tuned on HQ-Seg-44k) with that of Vanilla and SUM, both of which have undergone fine-tuning using the same lightweight scheme as HQ-SAM on FT-Small. The top row of this figure displays the mean IoU, while the bottom row shows the mean Boundary IoU. SUM matches HQ-SAM and outperforms Vanilla in salient-object segmentation, and is superior in entity and part segmentation.

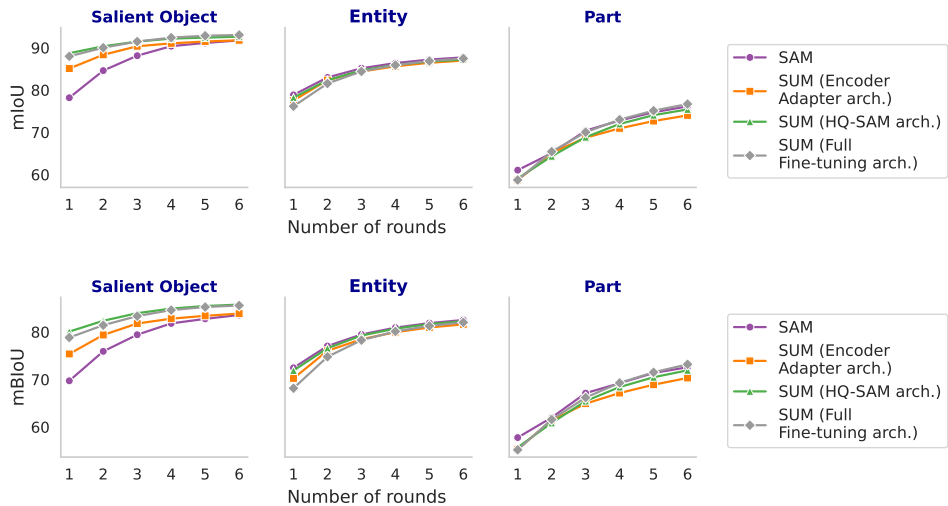

Figure 20: **Interactive segmentation evaluation with initial box prompt.** These results complement those in the bottom row of Fig. 4. Comparison of SAM and different versions of SUM fine-tuned on the FT-Small dataset. The top row of this figure displays the mean IoU, while the bottom row shows the mean boundary IoU.

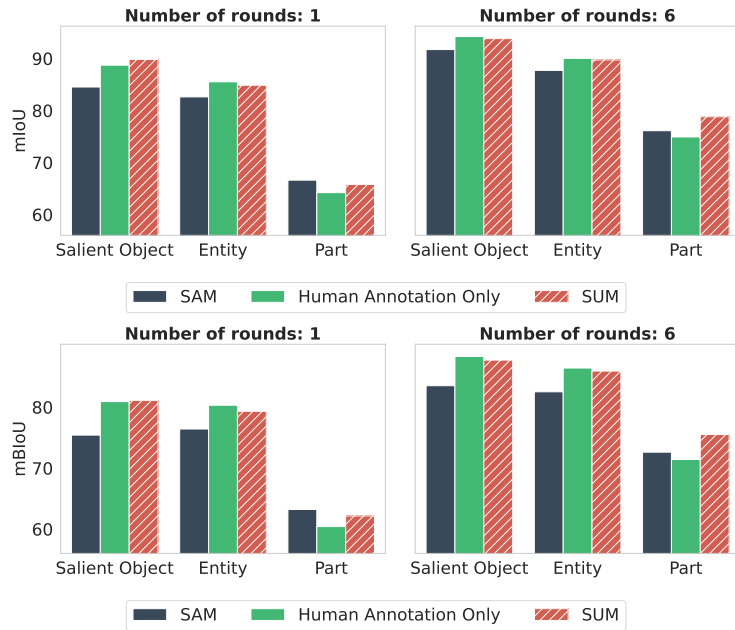

Figure 21: **Box-prompt assessment corresponding to Fig. 17.** This figure shows the interactive segmentation evaluation, where the first prompt is a box. We compare SAM, SUM fine-tuned on the FT-Medium dataset, and a Human-Annotation-Only baseline fine-tuned on a combination of the HQ-seg-44k and EntitySeg training sets. Fine-tuning only using human annotations improves performance for the tasks associated with the annotations (salient-object and entity segmentation), but deteriorates it for part segmentation in the case of one prompt. Since the box prompt has less semantic ambiguity compared to point prompts, the decrease is not as dramatic.

### G.2.2 Complementary Evaluation Metrics

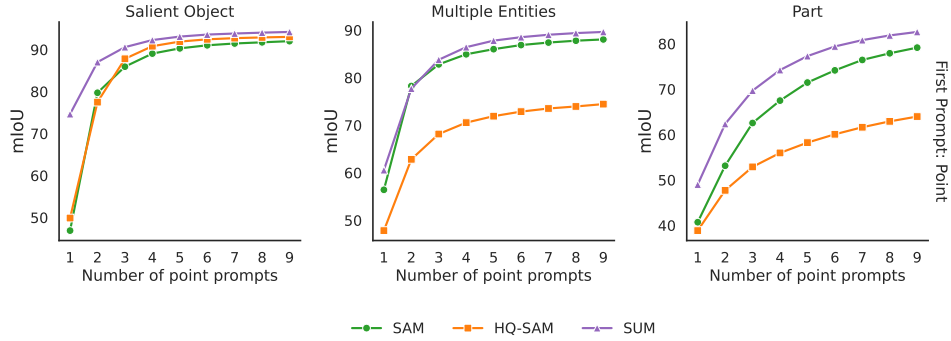

Figure 22: **Mean IoU complementary to the mean Boundary IoU in Fig. 1** SUM consistently outperforms SAM and HQ-SAM in quantitative comparisons, achieving the highest mean boundary IoU across diverse evaluation sets and interactive segmentation rounds. SUM model is fine-tuned on FT-Large dataset. Note that all the results here are single output results for a fair comparision with HQ-SAM.

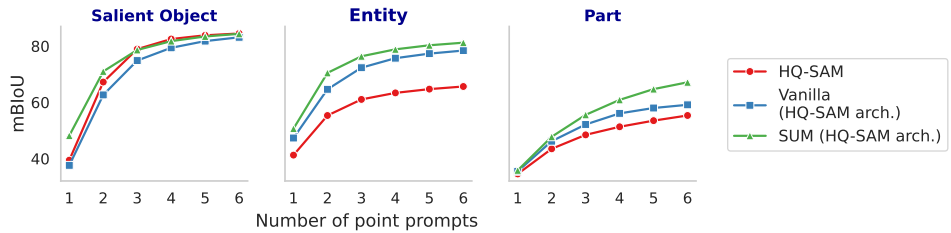

Figure 23: Mean boundary IoU supplementary to the mean IoU evaluation in Fig. 4. Comparison of HQ-SAM (fine-tuned on HQ-Seg-44k) with Vanilla and SUM fine-tuned using the same lightweight scheme as HQ-SAM. SUM matches HQ-SAM and outperforms Vanilla in salient-object segmentation, and is superior in entity and part segmentation.

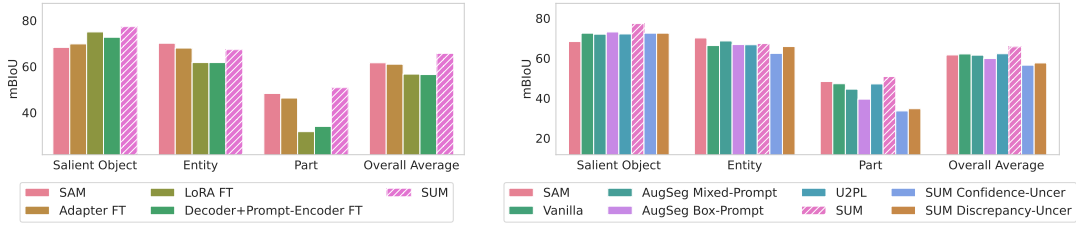

Figure 24: **Mean boundary IoU supplementary to the mean IoU evaluation in Fig. 5. Left:** Comparison of single point-prompt segmentation mean Boundary IoU for SUM versus models fine-tuned using various strategies on the HQSeg-44K dataset. All competing models improve on the salient-object segmentation task associated with this dataset but deteriorate on other segmentation tasks. **Right:** Mean Boundary IoU for single point-prompt segmentation: comparisons of models fine-tuned on FT-Small dataset with various strategies. SUM clearly outperforms all other strategies on salient-object and part segmentation, while preserving a competitive performance on entity segmentation.

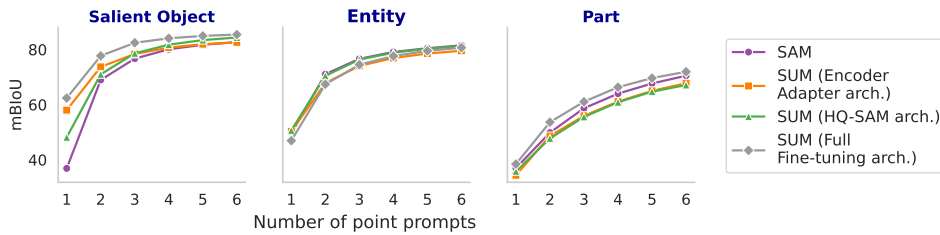

Figure 25: Mean boundary IoU supplementary to the mean IoU evaluation in Fig. 12. Comparison of SAM and different versions of SUM fine-tuned on the FT-Small dataset. Full parameter tuning shows the most significant improvement over SAM.

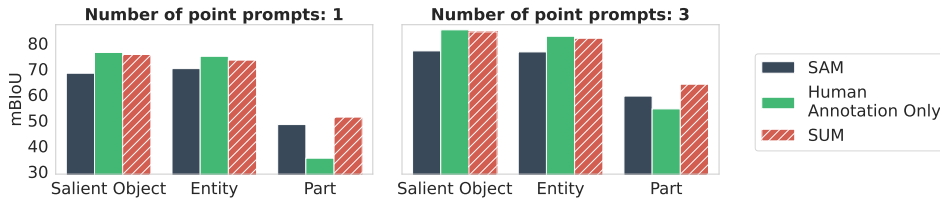

Figure 26: **Boundary IoU metrics corresponding to Fig. 17.** This figure shows the interactive segmentation mean boundary IoU of SAM, SUM fine-tuned on the FT-Medium dataset, and a Human-Annotation-Only baseline fine-tuned on a combination of the HQ-seg-44k and EntitySeg training sets. Fine-tuning only using human annotations improves performance for the tasks associated with the annotations (salient-object and entity segmentation), but deteriorates it dramatically for part segmentation in the case of one prompt.

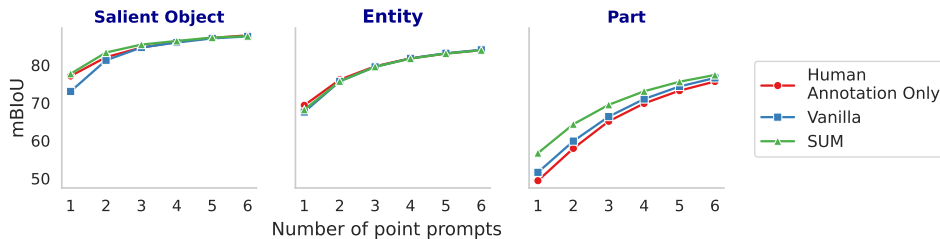

Figure 27: **Mean Boundary IoU evaluation corresponding to Fig. 18.** Comparison of the interactive segmentation mean boundary IoU of SUM and a Vanilla baseline fine-tuned on the FT-Large dataset, and a Human-Annotation-Only baseline fine-tuned on a large-scale internal dataset. SUM clearly outperforms the other models for part segmentation, and matches the performance of the Human-Annotation-Only baseline on the other tasks.

## G.3 Dataset-wise Comparison with HQ-SAM

Here, we provide the dataset-wise performance comparison of HQ-SAM (fine-tuned on HQ-Seg-44k) with Vanilla and SUM fine-tuned using the same lightweight scheme as HQ-SAM on FT-Small (HQ-Seg-44k and SA-250k).

We provide interactive segmentation results where the **first prompt is a point** in Table 10 (single-point-prompt segmentation) and Table 11 (six-point-prompt segmentation).

We provide interactive segmentation results where the **first prompt is a box** in Table 12 (single-box-prompt segmentation) and Table 13 (starting with a box prompt followed by five point prompts).

Table 10: **Single-point-prompt evaluation.** This table shows the comparison of HQ-SAM (fine-tuned on HQ-Seg-44k) with Vanilla and SUM fine-tuned using the same lightweight scheme as HQ-SAM. SUM is superior in all the tasks.

| Metrics | Task | Dataset | HQ-SAM | Vanilla (HQ-SAM arch.) | SUM (HQ-SAM arch.) |
|---|---|---|---|---|---|
| mIoU | Salient object | COIFT | 72.0 | 71.3 | **79.4** |
| | | DIS-VD | 23.8 | 21.0 | **29.0** |
| | | HRSOD | 58.4 | 51.5 | **65.9** |
| | | ThinObject5K | 44.6 | 45.2 | **54.7** |
| | | Big | 45.8 | 42.3 | **55.2** |
| | | Voceval | 54.5 | 51.0 | **60.3** |
| | | Average | 49.8 | 47.0 | **57.4** |
| | Entity | Entityseg 0 | 47.3 | 54.4 | **57.7** |
| | | Entityseg 1 | 48.5 | 53.6 | **56.7** |
| | | Entityseg 2 | 47.7 | 53.3 | **57.4** |
| | | Average | 47.8 | 53.8 | **57.3** |
| | Part | Bodypart | 56.7 | 57.3 | **59.2** |
| | | Easyportrait | 22.8 | **26.4** | 20.7 |
| | | Fashionpedia | 46.6 | 46.0 | **48.0** |
| | | Fashionpedia subpart | 29.9 | 29.3 | **30.7** |
| | | Paco | 38.1 | 38.6 | **39.3** |
| | | Average | 38.8 | 39.5 | **39.6** |
| | All | Average | 45.5 | 45.8 | **51.0** |
| mBIoU | Salient object | COIFT | 62.7 | 62.2 | **72.5** |
| | | DIS-VD | 18.6 | 16.0 | **23.4** |
| | | HRSOD | 48.9 | 42.2 | **57.7** |
| | | ThinObject5K | 32.9 | 34.1 | **43.3** |
| | | Big | 33.4 | 32.4 | **44.5** |
| | | Voceval | 40.2 | 38.7 | **47.6** |
| | | Average | 39.4 | 37.6 | **48.2** |
| | Entity | Entityseg 0 | 41.0 | 48.0 | **51.1** |
| | | Entityseg 1 | 42.3 | 47.8 | **50.6** |
| | | Entityseg 2 | 40.4 | 46.2 | **50.3** |
| | | Average | 41.2 | 47.3 | **50.7** |
| | Part | Bodypart | 50.2 | 51.0 | **53.4** |
| | | Easyportrait | 13.8 | **18.1** | 12.1 |
| | | Fashionpedia | 44.1 | 43.5 | **46.3** |
| | | Fashionpedia subpart | 28.1 | 27.7 | **29.5** |
| | | Paco | 36.6 | 37.2 | **37.9** |
| | | Average | 34.6 | 35.5 | **35.8** |
| | All | Average | 38.1 | 38.9 | **44.3** |

Table 11: **Six rounds interactive segmentation evaluation: first prompt with initial point prompt.** This table shows the comparison of HQ-SAM (fine-tuned on HQ-Seg-44k) with Vanilla and SUM fine-tuned using the same lightweight scheme as HQ-SAM. SUM matches HQ-SAM and outperforms Vanilla in salient-object segmentation, and is superior in entity and part segmentation.

| Metrics | Task | Dataset | HQ-SAM | Vanilla (HQ-SAM arch.) | SUM (HQ-SAM arch.) |
|---|---|---|---|---|---|
| mIoU | Salient object | COIFT | **95.8** | 95.5 | 95.6 |
| | | DIS-VD | **84.3** | 77.7 | 78.4 |
| | | HRSOD | **94.5** | 92.7 | 93.7 |
| | | ThinObject5K | **92.9** | 91.7 | 91.5 |
| | | Big | 94.0 | 93.5 | **95.7** |
| | | Voceval | 92.9 | 93.5 | **94.1** |
| | | Average | **92.4** | 90.8 | 91.5 |
| | Entity | Entityseg 0 | 70.5 | 81.5 | **84.1** |
| | | Entityseg 1 | 74.4 | 84.0 | **86.6** |
| | | Entityseg 2 | 73.6 | 85.9 | **88.4** |
| | | Average | 72.8 | 83.8 | **86.4** |
| | Part | Bodypart | 74.3 | 80.5 | **82.7** |
| | | Easyportrait | 50.1 | 47.6 | **61.2** |
| | | Fashionpedia | 64.9 | 68.6 | **74.4** |
| | | Fashionpedia subpart | 52.6 | 57.2 | **67.3** |
| | | Paco | 58.2 | 61.8 | **68.6** |
| | | Average | 60.0 | 63.1 | **70.8** |
| | All | Average | 76.7 | 79.4 | **83.0** |
| mBIoU | Salient object | COIFT | **91.6** | 91.3 | 91.4 |
| | | DIS-VD | **77.8** | 70.5 | 71.8 |
| | | HRSOD | **89.2** | 87.0 | 88.6 |
| | | ThinObject5K | **84.1** | 82.5 | 81.8 |
| | | Big | 82.9 | 83.9 | **88.0** |
| | | Voceval | 81.4 | 83.5 | **84.5** |
| | | Average | **84.5** | 83.1 | 84.4 |
| | Entity | Entityseg 0 | 63.2 | 76.0 | **78.8** |
| | | Entityseg 1 | 67.6 | 79.0 | **81.8** |
| | | Entityseg 2 | 66.1 | 80.2 | **83.0** |
| | | Average | 65.6 | 78.4 | **81.2** |
| | Part | Bodypart | 67.6 | 74.8 | **77.3** |
| | | Easyportrait | 40.0 | 38.5 | **52.7** |
| | | Fashionpedia | 62.1 | 66.4 | **72.5** |
| | | Fashionpedia subpart | 50.4 | 55.6 | **65.9** |
| | | Paco | 56.5 | 60.4 | **67.2** |
| | | Average | 55.3 | 59.1 | **67.1** |
| | All | Average | 70.0 | 73.5 | **77.5** |

Table 12: **Single-box-prompt evaluation.** This table shows the performance comparison of HQ-SAM (fine-tuned on HQ-Seg-44k) with Vanilla and SUM fine-tuned using the same lightweight scheme as HQ-SAM. SUM matches HQ-SAM and outperforms Vanilla in salient-object segmentation, and is superior in entity and part segmentation.

| Metrics | Task | Dataset | HQ-SAM | Vanilla (HQ-SAM arch.) | SUM (HQ-SAM arch.) |
|---|---|---|---|---|---|
| mIoU | Salient object | COIFT | **95.4** | 94.9 | 95.2 |
| | | DIS-VD | **79.5** | 70.4 | 77.1 |
| | | HRSOD | **92.9** | 90.3 | 92.7 |
| | | ThinObject5K | **89.8** | 86.6 | 88.7 |
| | | Big | 88.6 | 81.8 | **89.4** |
| | | Voceval | 88.2 | 87.2 | **89.0** |
| | | Average | **89.1** | 85.2 | 88.7 |
| | Entity | Entityseg 0 | 60.0 | 73.5 | **76.3** |
| | | Entityseg 1 | 64.6 | 75.8 | **78.8** |
| | | Entityseg 2 | 62.1 | 74.1 | **79.6** |
| | | Average | 62.2 | 74.5 | **78.2** |
| | Part | Bodypart | 67.9 | **68.5** | 68.0 |
| | | Easyportrait | **38.0** | 37.0 | 36.2 |
| | | Fashionpedia | 66.7 | 67.1 | **68.6** |
| | | Fashionpedia subpart | 55.2 | 56.7 | **62.5** |
| | | Paco | 56.2 | 56.3 | **59.9** |
| | | Average | 56.8 | 57.1 | **59.0** |
| | All | Average | 71.8 | 72.9 | **75.9** |
| mBIoU | Salient object | COIFT | **90.9** | 90.3 | **90.9** |
| | | DIS-VD | **71.6** | 62.7 | 69.0 |
| | | HRSOD | 86.1 | 83.0 | **86.8** |
| | | ThinObject5K | **78.8** | 75.4 | 77.8 |
| | | Big | 76.1 | 71.4 | **79.3** |
| | | Voceval | 74.7 | 74.4 | **76.5** |
| | | Average | 79.7 | 76.2 | **80.0** |
| | Entity | Entityseg 0 | 54.2 | 66.9 | **69.8** |
| | | Entityseg 1 | 58.8 | 69.6 | **73.1** |
| | | Entityseg 2 | 55.8 | 67.2 | **72.7** |
| | | Average | 56.3 | 67.9 | **71.9** |
| | Part | Bodypart | 61.8 | 63.0 | **63.6** |
| | | Easyportrait | **31.0** | **31.0** | 29.3 |
| | | Fashionpedia | 64.2 | 64.7 | **66.5** |
| | | Fashionpedia subpart | 53.2 | 54.7 | **60.8** |
| | | Paco | 54.9 | 54.8 | **58.4** |
| | | Average | 53.0 | 53.7 | **55.7** |
| | All | Average | 65.2 | 66.4 | **69.6** |

Table 13: **Six rounds interactive segmentation evaluation with initial box prompt.** This table shows the comparison of HQ-SAM (fine-tuned on HQ-Seg-44k) with Vanilla and SUM fine-tuned using the same lightweight scheme as HQ-SAM. SUM matches HQ-SAM and outperforms Vanilla in salient-object segmentation, and is superior in entity and part segmentation.

| Metrics | Task | Dataset | HQ-SAM | Vanilla (HQ-SAM arch.) | SUM (HQ-SAM arch.) |
|---|---|---|---|---|---|
| mIoU | Salient object | COIFT | **95.9** | 95.8 | 95.8 |
| | | DIS-VD | **85.9** | 82.6 | 82.4 |
| | | HRSOD | **94.7** | 93.5 | 94.1 |
| | | ThinObject5K | **93.2** | 92.6 | 92.6 |
| | | Big | 94.5 | 95.2 | **96.2** |
| | | Voceval | 92.1 | 93.5 | **94.7** |
| | | Average | **92.7** | 92.2 | 92.6 |
| | Entity | Entityseg 0 | 68.7 | 82.2 | **85.2** |
| | | Entityseg 1 | 72.2 | 84.6 | **87.5** |
| | | Entityseg 2 | 70.7 | 84.6 | **89.0** |
| | | Average | 70.5 | 83.8 | **87.2** |
| | Part | Bodypart | 79.3 | 82.6 | **84.5** |
| | | Easyportrait | 50.8 | 54.0 | **66.0** |
| | | Fashionpedia | 72.3 | 73.3 | **78.4** |
| | | Fashionpedia subpart | 62.4 | 64.9 | **74.4** |
| | | Paco | 62.5 | 65.4 | **73.8** |
| | | Average | 65.5 | 68.0 | **75.4** |
| | All | Average | 78.2 | 81.8 | **85.3** |
| mBIoU | Salient object | COIFT | 91.5 | **91.7** | 91.6 |
| | | DIS-VD | **79.4** | 75.6 | 76.2 |
| | | HRSOD | **89.6** | 87.9 | 89.3 |
| | | ThinObject5K | **84.3** | 83.2 | 83.1 |
| | | Big | 83.1 | 86.0 | **88.8** |
| | | Voceval | 80.8 | 83.7 | **85.4** |
| | | Average | 84.8 | 84.7 | **85.7** |
| | Entity | Entityseg 0 | 62.2 | 76.9 | **80.1** |
| | | Entityseg 1 | 65.9 | 79.9 | **82.9** |
| | | Entityseg 2 | 63.8 | 79.3 | **83.8** |
| | | Average | 64.0 | 78.7 | **82.3** |
| | Part | Bodypart | 72.5 | 77.1 | **79.5** |
| | | Easyportrait | 41.7 | 45.3 | **57.8** |
| | | Fashionpedia | 69.4 | 71.2 | **76.7** |
| | | Fashionpedia subpart | 60.2 | 63.3 | **73.2** |
| | | Paco | 61.0 | 64.1 | **72.5** |
| | | Average | 61.0 | 64.2 | **71.9** |
| | All | Average | 71.8 | 76.1 | **80.1** |

## G.4 Dataset-wise Comparison with SAM

Here, we provide the dataset-wise performance comparison of SAM with SUM fine-tuned in a full parameter fine-tuning manner on different train sets: FT-Small, FT-Medium, and FT-Large.

We provide interactive segmentation results where the **first prompt is a point** in Table 14 (single-point-prompt segmentation), Table 15 (five-point-prompt segmentation) and Table 16 (six-point-prompt segmentation).

We provide interactive segmentation results where the **first prompt is a box** in Table 17 (single-box-prompt segmentation) and Table 18 (Prompts: a box as first prompt and following 5 point prompts).

Table 14: **Single-point-prompt evaluation** Comparison of SAM with SUM fine-tuned on FT-Small, FT-Medium, and FT-Large. SUM achieves consistent overall improvement compared to SAM, it improves more when more human-annotated data is included.

| Metrics | Task | Dataset | SAM | SUM (FT-small) | SUM (FT-Medium) | SUM (FT-Large) |
|---|---|---|---|---|---|---|
| mIoU | Salient object | COIFT | 93.7 | **95.0** | 93.1 | 94.8 |
| | | DIS-VD | 51.0 | 72.6 | 69.1 | **74.1** |
| | | HRSOD | 85.1 | 91.9 | 90.1 | **92.1** |
| | | ThinObject5K | 79.4 | **90.2** | 87.5 | 88.1 |
| | | Big | 81.6 | 83.6 | 86.6 | **87.0** |
| | | Voceval | 81.1 | 85.4 | 84.8 | **85.5** |
| | | Average | 78.7 | 86.4 | 85.2 | **86.9** |
| | Entity | Entityseg 0 | 75.4 | 73.7 | **78.1** | 74.9 |
| | | Entityseg 1 | 77.2 | 74.9 | **79.3** | 75.3 |
| | | Entityseg 2 | 79.6 | 76.8 | **81.9** | 77.1 |
| | | Average | 77.4 | 75.1 | **79.8** | 75.8 |
| | Part | Bodypart | 66.6 | 65.8 | 65.9 | **68.5** |
| | | Easyportrait | 40.6 | 56.4 | 55.3 | **75.1** |
| | | Fashionpedia | 58.6 | 58.2 | 58.4 | **61.8** |
| | | Fashionpedia subpart | 44.7 | 44.2 | 45.3 | **46.8** |
| | | Paco | 50.7 | 50.6 | **51.8** | 50.5 |
| | | Average | 52.2 | 55.0 | 55.3 | **60.5** |
| | All | Average | 69.0 | 72.8 | 73.4 | **75.1** |
| mBIoU | Salient object | COIFT | 88.0 | **90.9** | 88.4 | 90.4 |
| | | DIS-VD | 41.3 | 64.7 | 60.8 | **66.6** |
| | | HRSOD | 77.3 | 85.2 | 83.0 | **86.1** |
| | | ThinObject5K | 64.5 | **79.7** | 76.1 | 77.7 |
| | | Big | 70.9 | 72.5 | **75.2** | 74.9 |
| | | Voceval | 67.7 | **70.9** | 70.0 | 70.6 |
| | | Average | 68.3 | 77.3 | 75.6 | **77.7** |
| | Entity | Entityseg 0 | 68.3 | 66.1 | **71.8** | 67.6 |
| | | Entityseg 1 | 70.4 | 67.5 | **73.2** | 68.1 |
| | | Entityseg 2 | 71.5 | 68.2 | **75.0** | 68.9 |
| | | Average | 70.1 | 67.3 | **73.3** | 68.2 |
| | Part | Bodypart | 60.6 | 59.3 | 59.4 | **61.7** |
| | | Easyportrait | 32.6 | 47.9 | 47.4 | **69.5** |
| | | Fashionpedia | 56.5 | 55.8 | 55.9 | **59.2** |
| | | Fashionpedia subpart | 42.9 | 42.3 | 43.3 | **44.8** |
| | | Paco | 49.0 | 48.7 | **49.9** | 48.5 |
| | | Average | 48.3 | 50.8 | 51.2 | **56.7** |
| | All | Average | 61.6 | 65.7 | 66.4 | **68.2** |

Table 15: **Five rounds interactive segmentation evaluation with initial point prompt.** Comparison of SAM with SUM fine-tuned on FT-Small, FT-Medium, and FT-Large. SUM consistently outperforms SAM, showing even greater improvement as the budget of human-annotated data increases.

| Metrics | Task | Dataset | SAM | SUM (FT-small) | SUM (FT-Medium) | SUM (FT-Large) |
|---|---|---|---|---|---|---|
| | | COIFT | 95.2 | 96.0 | 96.1 | **96.4** |
| | | DIS-VD | 76.2 | 84.1 | **84.3** | 84.1 |
| | | HRSOD | 92.7 | 93.7 | 94.2 | **94.8** |
| | Salient object | ThinObject5K | 90.1 | 92.6 | 93.3 | **93.5** |
| | | Big | 94.1 | 94.8 | 96.6 | **96.9** |
| | | Voceval | 93.2 | 93.8 | 94.6 | **94.7** |
| | | Average | 90.3 | 92.5 | 93.2 | **93.4** |
| | | Entityseg 0 | 83.7 | 84.0 | **87.0** | 86.3 |
| | Entity | Entityseg 1 | 86.3 | 86.2 | **88.7** | 87.8 |
| | | Entityseg 2 | 88.3 | 88.1 | **90.4** | 89.3 |
| mIoU | | Average | 86.1 | 86.1 | **88.7** | 87.8 |
| | | Bodypart | 78.3 | 81.9 | 82.3 | **85.6** |
| | | Easyportrait | 69.6 | 77.7 | 76.1 | **86.2** |
| | Part | Fashionpedia | 75.8 | 75.2 | 76.4 | **78.1** |
| | | Fashionpedia subpart | 67.6 | 68.5 | 70.1 | **70.6** |
| | | Paco | 68.1 | 68.7 | 70.3 | **71.7** |
| | | Average | 71.9 | 74.4 | 75.0 | **78.4** |
| | All | Average | 82.8 | 84.7 | 85.7 | **86.9** |
| | | COIFT | 90.9 | 92.3 | 92.7 | **93.2** |
| | | DIS-VD | 68.0 | 77.4 | 78.5 | **78.6** |
| | | HRSOD | 87.1 | 88.4 | 89.8 | **90.9** |
| | Salient object | ThinObject5K | 79.0 | 84.0 | 85.6 | **86.2** |
| | | Big | 85.2 | 85.6 | 89.1 | **90.1** |
| | | Voceval | 81.3 | 82.2 | 84.5 | **84.8** |
| | | Average | 81.9 | 85.0 | 86.7 | **87.3** |
| | | Entityseg 0 | 78.1 | 78.2 | **82.9** | 81.6 |
| | Entity | Entityseg 1 | 81.3 | 80.9 | **84.8** | 83.4 |
| | | Entityseg 2 | 82.5 | 82.1 | **86.1** | 84.2 |
| mBIoU | | Average | 80.6 | 80.4 | **84.6** | 83.1 |
| | | Bodypart | 73.1 | 76.4 | 76.9 | **80.3** |
| | | Easyportrait | 60.7 | 69.5 | 67.2 | **81.4** |
| | Part | Fashionpedia | 74.0 | 73.2 | 74.7 | **76.6** |
| | | Fashionpedia subpart | 66.3 | 67.0 | 68.7 | **69.4** |
| | | Paco | 66.7 | 67.2 | 69.0 | **70.4** |
| | | Average | 68.2 | 70.7 | 71.3 | **75.6** |
| | All | Average | 76.7 | 78.9 | 80.8 | **82.2** |

Table 16: **Six rounds interactive segmentation evaluation with initial point prompt.** Comparison of SAM with SUM fine-tuned on FT-Small, FT-Medium, and FT-Large. Comparison of SAM with SUM fine-tuned on FT-Small, FT-Medium, and FT-Large. SUM consistently outperforms SAM, showing even greater improvement as the budget of human-annotated data increases.

| Metrics | Task | Dataset | SAM | SUM (FT-small) | SUM (FT-Medium) | SUM (FT-Large) |
|---|---|---|---|---|---|---|
| mIoU | Salient object | COIFT | 95.4 | 95.9 | 96.1 | **96.5** |
| | | DIS-VD | 78.2 | 84.6 | **85.4** | 84.8 |
| | | HRSOD | 93.5 | 93.8 | 94.3 | **95.0** |
| | | ThinObject5K | 91.0 | 92.8 | 93.5 | **93.6** |
| | | Big | 95.4 | 95.5 | 96.9 | **97.2** |
| | | Voceval | 93.6 | 94.4 | 94.9 | **95.0** |
| | | Average | 91.2 | 92.8 | 93.5 | **93.7** |
| | Entity | Entityseg 0 | 84.6 | 84.8 | **87.5** | 86.8 |
| | | Entityseg 1 | 87.1 | 86.9 | **89.2** | 88.5 |
| | | Entityseg 2 | 88.8 | 88.7 | **90.9** | 90.0 |
| | | Average | 86.8 | 86.8 | **89.2** | 88.4 |
| | Part | Bodypart | 80.1 | 83.8 | 84.1 | **86.6** |
| | | Easyportrait | 73.9 | 79.9 | 78.5 | **87.2** |
| | | Fashionpedia | 77.6 | 76.7 | 78.0 | **79.7** |
| | | Fashionpedia subpart | 70.4 | 70.7 | 72.4 | **72.8** |
| | | Paco | 70.4 | 70.5 | 72.3 | **73.9** |
| | | Average | 74.5 | 76.3 | 77.1 | **80.0** |
| | All | Average | 84.3 | 85.6 | 86.7 | **87.7** |
| mBIoU | Salient object | COIFT | 91.2 | 92.3 | 92.8 | **93.3** |
| | | DIS-VD | 70.2 | 77.9 | **79.9** | 79.4 |
| | | HRSOD | 88.0 | 88.5 | 90.0 | **91.0** |
| | | ThinObject5K | 79.8 | 84.3 | 86.0 | **86.5** |
| | | Big | 86.4 | 86.4 | 90.0 | **90.3** |
| | | Voceval | 82.1 | 83.1 | 85.1 | **85.4** |
| | | Average | 82.9 | 85.4 | 87.3 | **87.6** |
| | Entity | Entityseg 0 | 79.1 | 79.2 | **83.5** | 82.4 |
| | | Entityseg 1 | 82.2 | 81.7 | **85.5** | 84.2 |
| | | Entityseg 2 | 83.3 | 82.8 | **86.8** | 85.2 |
| | | Average | 81.5 | 81.2 | **85.3** | 83.9 |
| | Part | Bodypart | 75.2 | 78.5 | 78.9 | **81.5** |
| | | Easyportrait | 65.3 | 71.8 | 70.1 | **82.8** |
| | | Fashionpedia | 75.9 | 74.8 | 76.4 | **78.3** |
| | | Fashionpedia subpart | 69.2 | 69.2 | 71.1 | **71.7** |
| | | Paco | 69.1 | 69.1 | 71.0 | **72.7** |
| | | Average | 70.9 | 72.7 | 73.5 | **77.4** |
| | All | Average | 78.3 | 80.0 | 81.9 | **83.2** |

Table 17: **Single-box-prompt segmentation comparison** SUM consistently outperforms SAM, showing even greater improvement as the amount of human-annotated data increases.

| Metrics | Task | Dataset | SAM | SUM (FT-small) | SUM (FT-Medium) | SUM (FT-Large) |
|---|---|---|---|---|---|---|
| mIoU | Salient object | COIFT | 94.3 | **95.7** | 95.0 | 95.6 |
| | | DIS-VD | 68.3 | **79.7** | 79.2 | 78.8 |
| | | HRSOD | 91.3 | 93.0 | 92.8 | **93.5** |
| | | ThinObject5K | 80.0 | **91.5** | 89.6 | 86.8 |
| | | Big | 85.8 | 90.5 | **91.5** | 90.9 |
| | | Voceval | 87.3 | 89.7 | **90.7** | 89.5 |
| | | Average | 84.5 | **90.0** | 89.8 | 89.2 |
| | Entity | Entityseg 0 | 80.8 | 79.7 | **83.1** | 81.7 |
| | | Entityseg 1 | 82.7 | 81.4 | **84.9** | 83.2 |
| | | Entityseg 2 | 84.4 | 82.8 | **86.5** | 83.6 |
| | | Average | 82.6 | 81.3 | **84.8** | 82.8 |
| | Part | Bodypart | 74.5 | 73.7 | 74.7 | **75.8** |
| | | Easyportrait | 47.3 | 46.4 | 48.8 | **57.8** |
| | | Fashionpedia | **74.8** | 72.3 | 72.5 | 74.3 |
| | | Fashionpedia subpart | **69.6** | 66.8 | 67.3 | 68.2 |
| | | Paco | 67.0 | 64.9 | 65.6 | **67.2** |
| | | Average | 66.6 | 64.8 | 65.8 | **68.7** |
| | All | Average | 77.7 | 79.2 | 80.2 | **80.5** |
| mBIoU | Salient object | COIFT | 89.6 | **92.1** | 90.9 | 91.4 |
| | | DIS-VD | 59.9 | **72.0** | 71.8 | 71.8 |
| | | HRSOD | 85.0 | 87.2 | 86.8 | **88.0** |
| | | ThinObject5K | 67.1 | **81.4** | 78.8 | 76.4 |
| | | Big | 76.0 | 79.9 | **80.9** | 80.7 |
| | | Voceval | 74.9 | 76.6 | **77.6** | 76.6 |
| | | Average | 75.4 | **81.5** | 81.1 | 80.8 |
| | Entity | Entityseg 0 | 74.5 | 72.9 | **77.6** | 75.6 |
| | | Entityseg 1 | 77.1 | 74.9 | **79.6** | 77.4 |
| | | Entityseg 2 | 77.6 | 75.2 | **80.6** | 76.8 |
| | | Average | 76.4 | 74.3 | **79.3** | 76.6 |
| | Part | Bodypart | **69.3** | 68.0 | 68.9 | **69.3** |
| | | Easyportrait | 40.3 | 39.0 | 41.8 | **53.3** |
| | | Fashionpedia | **72.8** | 70.1 | 70.4 | 72.1 |
| | | Fashionpedia subpart | **68.0** | 65.1 | 65.6 | 66.6 |
| | | Paco | **65.5** | 63.2 | 64.0 | **65.5** |
| | | Average | 63.2 | 61.1 | 62.1 | **65.4** |
| | All | Average | 71.3 | 72.7 | 73.9 | **74.4** |

Table 18: **Six rounds interactive segmentation evaluation with initial box prompt.** Comparison of SAM with SUM fine-tuned on FT-Small, FT-Medium, and FT-Large. SUM consistently outperforms SAM, showing greater improvement as the amount of human-annotated data increases.

| Metrics | Task | Dataset | SAM | SUM (FT-small) | SUM (FT-Medium) | SUM (FT-Large) |
|---|---|---|---|---|---|---|
| mIoU | Salient object | COIFT | 95.7 | 96.1 | 96.1 | **96.4** |
| | | DIS-VD | 80.3 | 85.3 | **86.6** | 86.1 |
| | | HRSOD | 93.7 | 93.9 | 94.3 | **95.1** |
| | | ThinObject5K | 91.2 | 93.0 | 93.7 | **93.9** |
| | | Big | 95.5 | 95.6 | 97.0 | **97.3** |
| | | Voceval | 94.0 | 94.5 | 95.1 | **95.3** |
| | | Average | 91.7 | 93.1 | 93.8 | **94.0** |
| | Entity | Entityseg 0 | 85.8 | 85.7 | **88.1** | 87.7 |
| | | Entityseg 1 | 88.0 | 87.6 | **89.7** | 89.2 |
| | | Entityseg 2 | 89.4 | 89.1 | **91.3** | 90.6 |
| | | Average | 87.7 | 87.5 | **89.7** | 89.2 |
| | Part | Bodypart | 82.8 | 84.4 | 85.7 | **87.3** |
| | | Easyportrait | 63.9 | 71.6 | 76.8 | **86.4** |
| | | Fashionpedia | 81.0 | 78.8 | 79.9 | **81.2** |
| | | Fashionpedia subpart | 77.6 | 75.2 | 76.2 | **78.2** |
| | | Paco | 75.3 | 73.8 | 75.3 | **77.5** |
| | | Average | 76.1 | 76.8 | 78.8 | **82.1** |
| | All | Average | 85.3 | 86.0 | 87.6 | **88.7** |
| mBIoU | Salient object | COIFT | 91.4 | 92.5 | 92.7 | **93.2** |
| | | DIS-VD | 72.5 | 78.5 | **81.1** | 80.6 |
| | | HRSOD | 88.3 | 88.6 | 90.0 | **91.2** |
| | | ThinObject5K | 79.8 | 84.4 | 86.2 | **86.7** |
| | | Big | 86.5 | 86.2 | **90.5** | 90.5 |
| | | Voceval | 82.7 | 83.3 | 85.5 | **85.9** |
| | | Average | 83.5 | 85.6 | 87.7 | **88.0** |
| | Entity | Entityseg 0 | 80.4 | 80.3 | **84.2** | 83.4 |
| | | Entityseg 1 | 83.1 | 82.5 | **86.2** | 85.2 |
| | | Entityseg 2 | 84.0 | 83.3 | **87.3** | 86.0 |
| | | Average | 82.5 | 82.0 | **85.9** | 84.9 |
| | Part | Bodypart | 77.7 | 79.2 | 80.9 | **82.5** |
| | | Easyportrait | 55.3 | 63.9 | 69.0 | **82.5** |
| | | Fashionpedia | 79.4 | 76.9 | 78.3 | **79.7** |
| | | Fashionpedia subpart | 76.4 | 73.8 | 75.0 | **77.2** |
| | | Paco | 74.0 | 72.4 | 74.1 | **76.3** |
| | | Average | 72.6 | 73.2 | 75.5 | **79.6** |
| | All | Average | 79.4 | 80.4 | 82.9 | **84.4** |

### G.5 SUM vs SAM under the Single Output Evaluation Setting

In this section, we provide a single output evaluation of SAM and SUM fine-tuned on different train sets (FT-Small, FT-Medium, and FT-Large). We provide the IoU evaluation in Fig. 28 and boundary IoU evaluation in Fig. 29. In this setting, only one output is provided when a single prompt is provided, as specified in Appendix B.3.

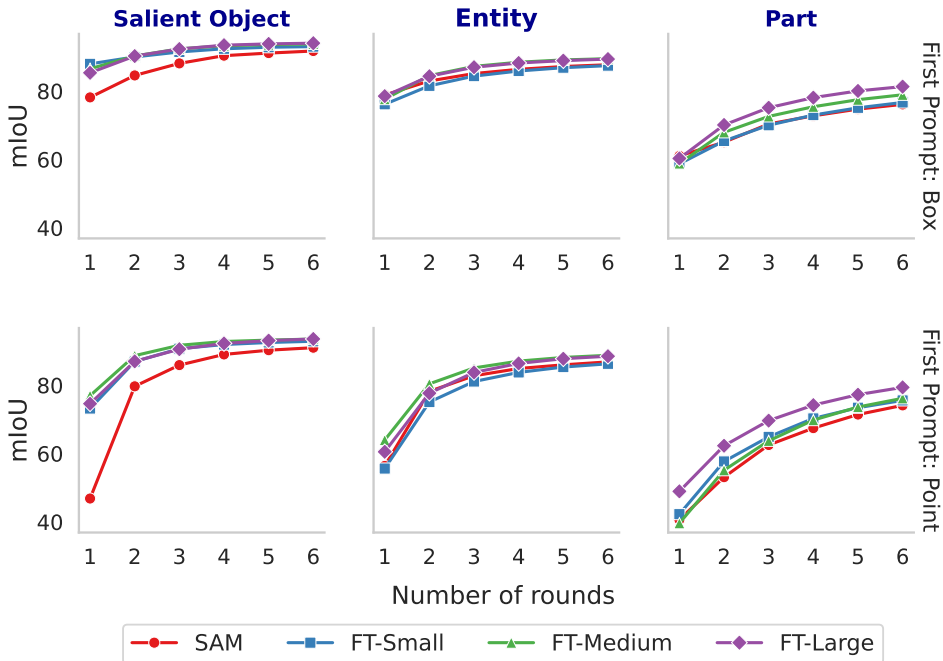

Figure 28: **Evaluation of mean IoU for the single output.** This figure displays the mIoU values for the single output case, segregated into three segmentation tasks: Salient Object, Entity, and Part. The performance of the SAM and models fine-tuned on datasets of varying annotation budgets —small (FT-Small), medium (FT-Medium), and large (FT-Large)—is plotted over six iterative rounds.

### G.6 Results on Additional Evaluation Sets

We organize our evaluation tasks in the experiment according to the hierarchical level of granularity, covering various levels (part, entity, multiple instances). Each task is evaluated using several diverse datasets designed to encompass a wide range of images and subtasks. For instance, our part segmentation evaluation utilizes five diverse datasets: Fashionpedia, Fashionpedia Subpart, Paco, Multi-Human Parsing, and Portrait. The first two include a comprehensive ontology of fashion elements and different levels of part granularity. Paco covers 75 object classes and the last two focus on different human-specific part segmentation subtasks.

To further strengthen the robustness of our proposed methods, we have extended our evaluation by testing SUM and SAM on additional image types. For reproducibility, SUM is fine-tuned on the Public dataset FT-Medium. We selected 7 datasets from the evaluation sets used in SAM to complement our existing 14 public evaluation sets. Additionally, we have included part one of a synthetic dataset, GTAV [82]. These additional evaluation sets encompass various image types e.g. driving, synthetic, egocentric, irregular shapes, paintings, underwater animals, drones, and underwater trash.

The mIoU comparison results, reported in the Table 19, confirm that SUM consistently outperforms SAM.

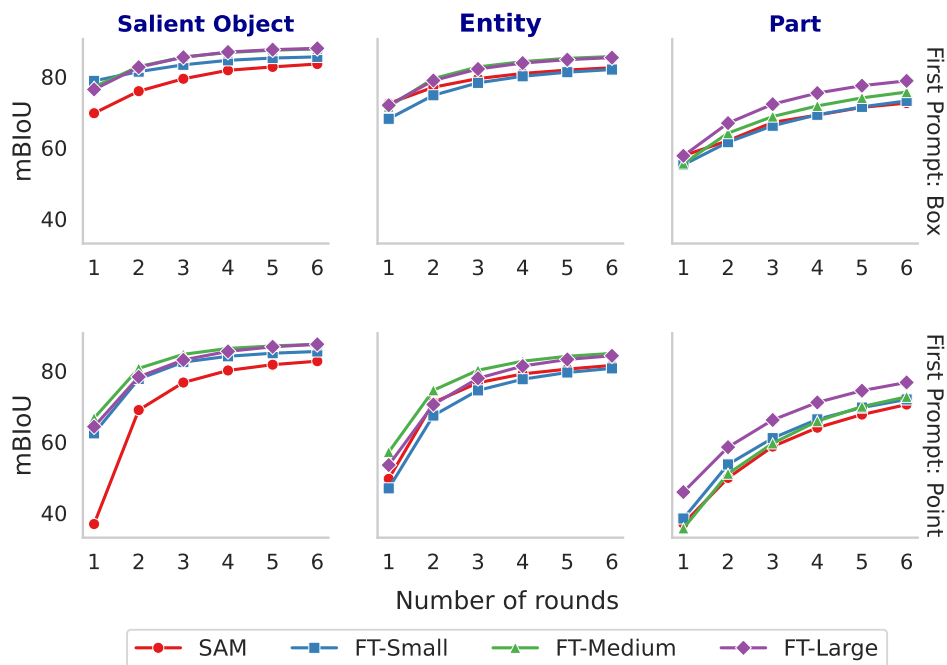

Figure 29: **Evaluation of mean boundary IoU for the single output.** This figure displays the mean boundary IoU values for the single output cases, segregated into three segmentation tasks: Salient Object, Entity, and Part. The performance of the SAM and models fine-tuned on datasets of varying annotation budgets —small (FT-Small), medium (FT-Medium), and large (FT-Large)—is plotted over six iterative rounds.

Table 19: **Point-prompted interactive segmentation** This table reports the point-prompt mean IoU over interactive segmentation rounds. SUM consistently outperforms SAM across additional evaluation datasets over interactive rounds.

| Dataset | Image type | Method | Round 1 | Round 3 | Round 5 | Round 9 |
|---|---|---|---|---|---|---|
| Cityscapes [83] | Driving | SAM | 44.1 | 50.9 | 58.6 | 64.6 |
| | | SUM | **46.4** | **57.3** | **62.5** | **67.1** |
| EgoHOS [84] | Egocentric | SAM | 77.9 | 85.4 | 90.2 | 91.9 |
| | | SUM | **79.0** | **90.0** | **92.3** | **93.5** |
| DRAM [85] | Paintings | SAM | 68.4 | 81.3 | 87.9 | 91.1 |
| | | SUM | **73.6** | **85.8** | **89.2** | **91.4** |
| ishape [86] | Irregular shapes | SAM | 41.4 | 60.4 | 75.7 | 84.6 |
| | | SUM | **74.9** | **87.3** | **92.6** | **94.4** |
| GTAV [82] | Synthetic | SAM | 44.8 | 47.0 | 52.2 | 57.0 |
| | | SUM | **45.7** | **52.8** | **56.7** | **59.7** |
| NDD20 [87] | Underwater Animal | SAM | 86.2 | 88.3 | 90.2 | 91.3 |
| | | SUM | **87.9** | **91.3** | **92.2** | **93.1** |
| TrashCan [88] | Underwater | SAM | 63.3 | 69.1 | 76.7 | 82.6 |
| | | SUM | **64.9** | **76.8** | **82.0** | **86.1** |
| IBD [89] | Drones | SAM | **78.8** | 84.5 | 91.2 | 93.6 |
| | | SUM | 78.4 | **87.6** | **91.4** | **93.9** |

# H Additional Qualitative Results

## H.1 Task Prompt

Fig. 30, 31, 32, and 33 shows the multiple outputs of SUM for the same input and point prompt when we vary the task prompt. When provided with task prompt 1, associated with 'Salient Object', the model is able to generate segmentation masks that encompass multiple salient objects within the image.

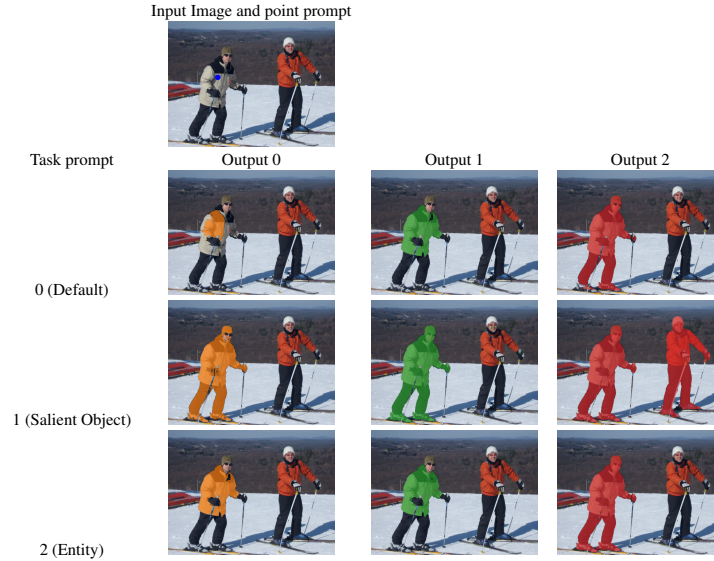

Figure 30: Outputs of SUM for different task prompts.

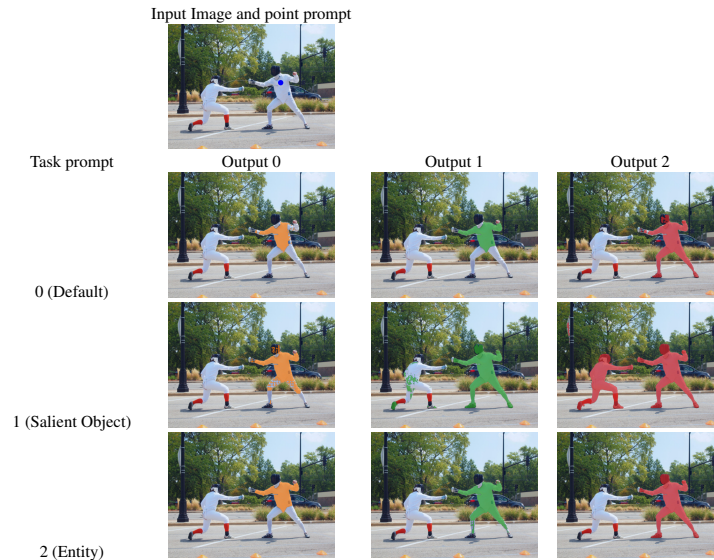

Figure 31: Outputs of SUM for different task prompts.

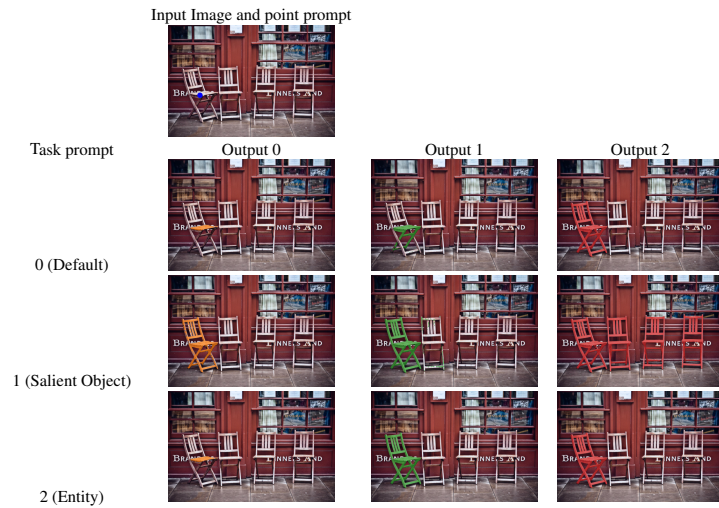

Figure 32: Outputs of SUM for different task prompts.

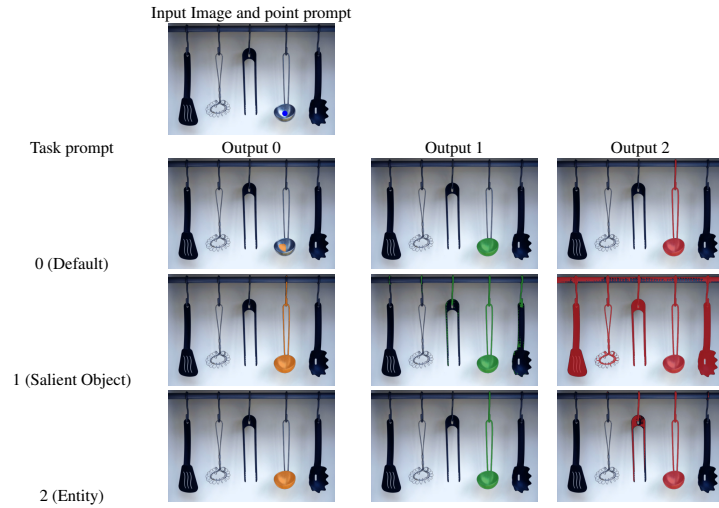

Figure 33: Outputs of SUM for different task prompts.

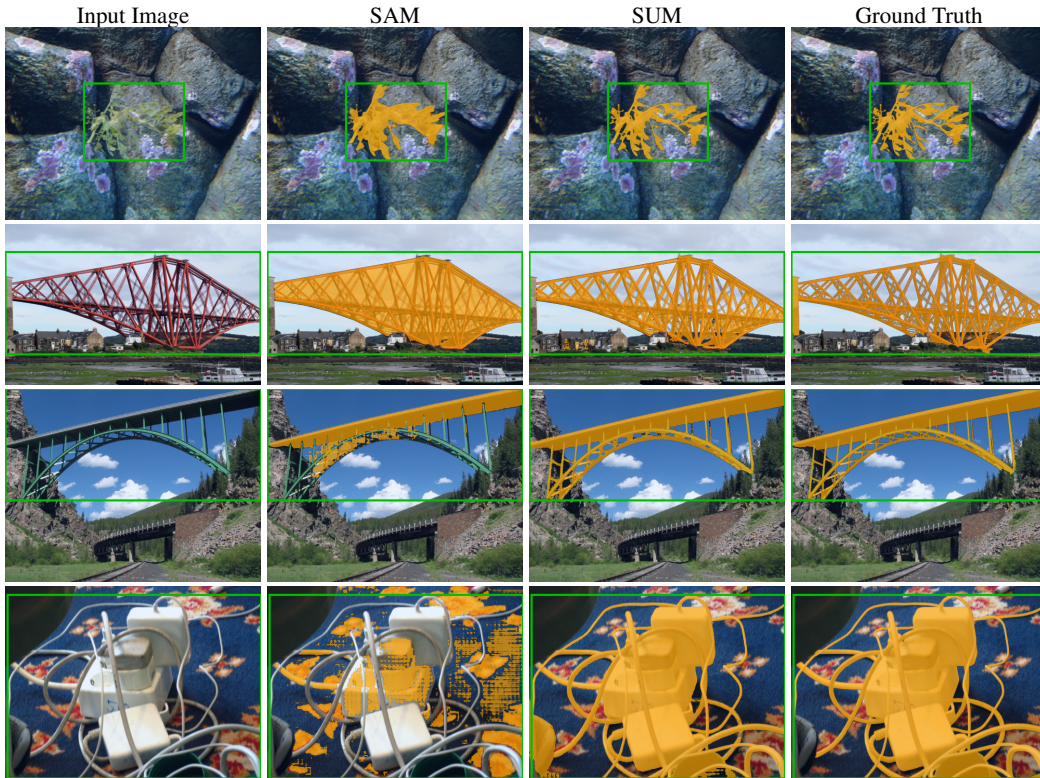

Figure 34: Comparative visualization of segmentation outcomes using single-box prompts. SUM produces predictions of substantially higher quality than those of SAM.

## H.2    SUM versus SAM

The following examples compare SUM to SAM for different segmentation tasks: salient object segmentation in Fig 34, Fig. 35 and Fig. 36; entity segmentation in Fig. 37; and part segmentation in Fig. 38.

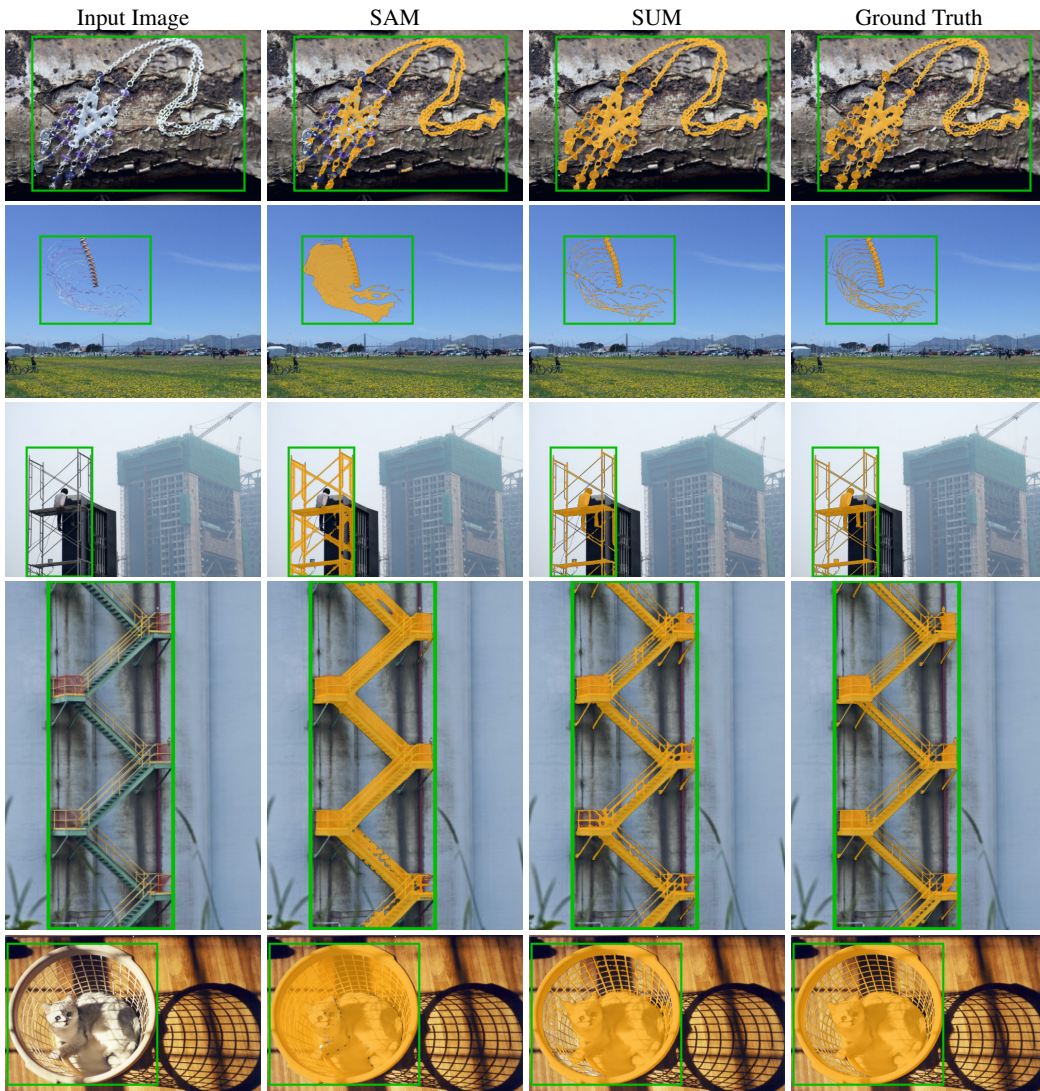

Figure 35: Comparative visualization of segmentation outcomes using single-box prompts. SUM produces predictions of substantially higher quality than those of SAM.

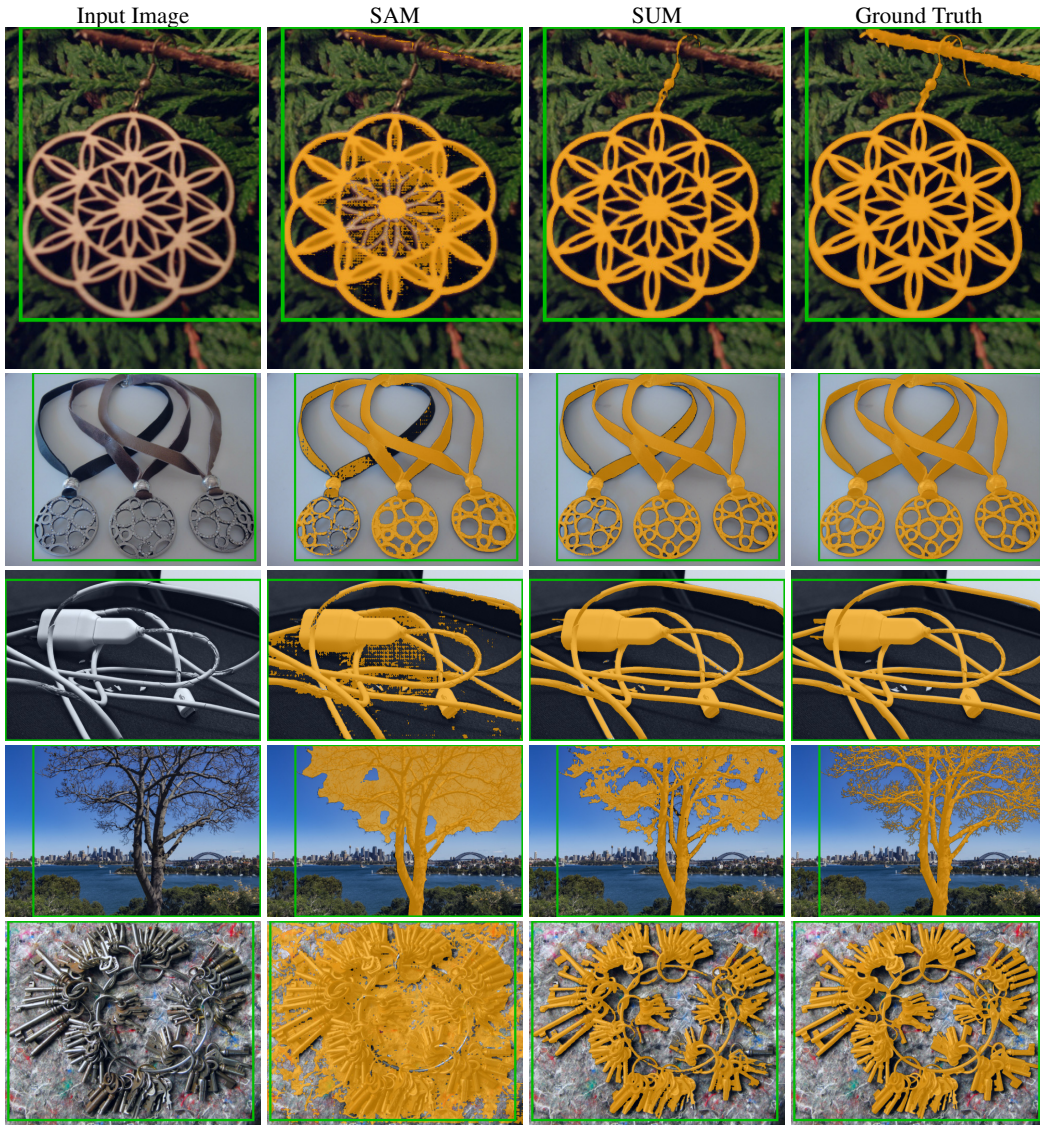

Figure 36: Comparative visualization of segmentation outcomes using single-box prompts. SUM produces predictions of substantially higher quality than those of SAM.

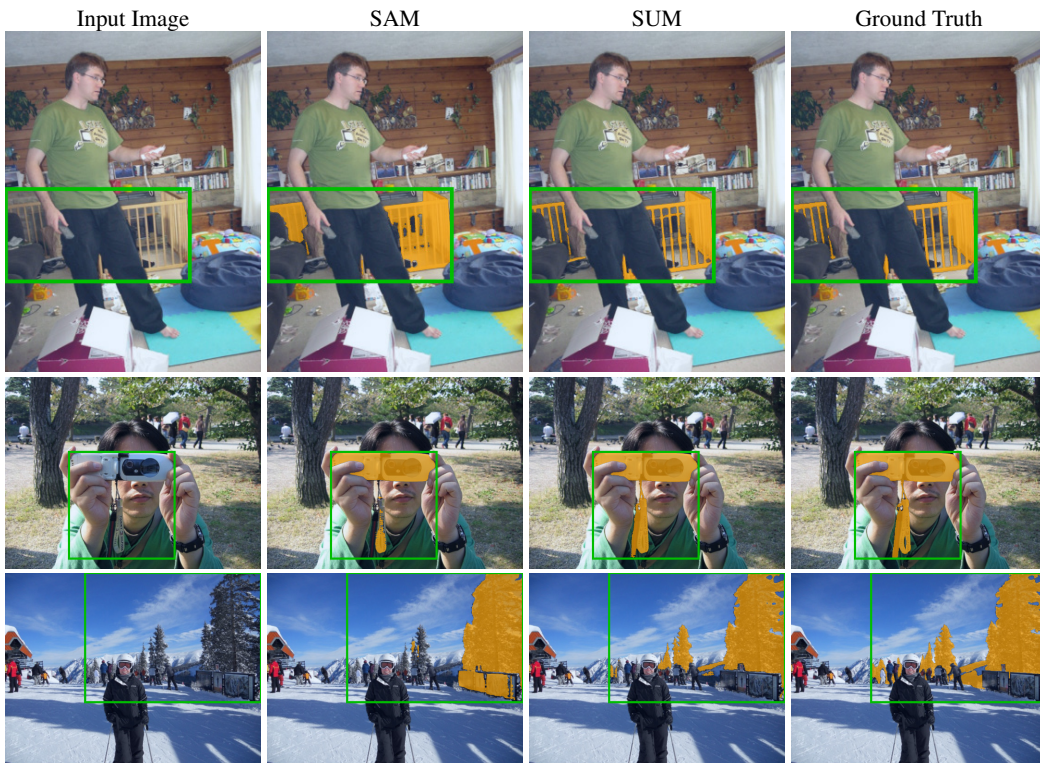

Figure 37: Comparative visualization of segmentation outcomes using single-box prompts. SUM produces predictions of substantially higher quality than those of SAM.

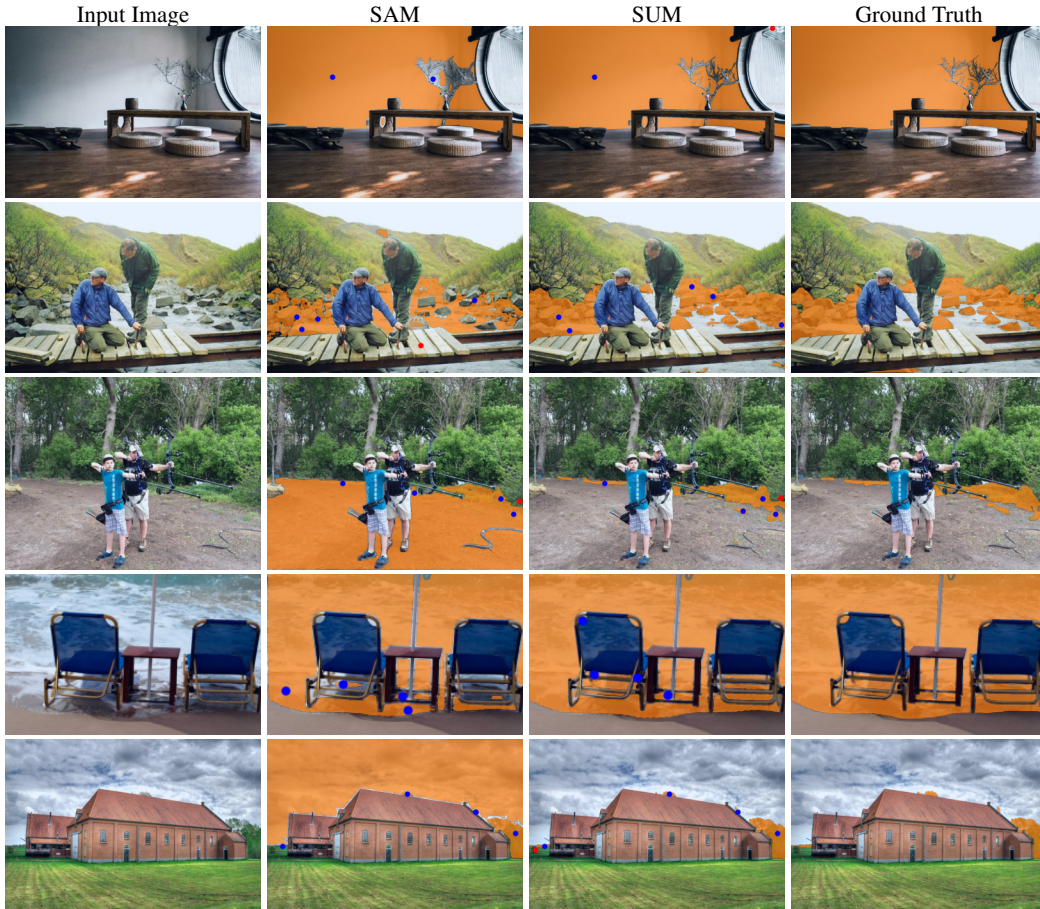

Figure 38: Comparative visualization of segmentation outcomes using point prompts, where blue points signify positive prompts and red points indicate negative prompts. SUM produces predictions of substantially higher quality than those of SAM.

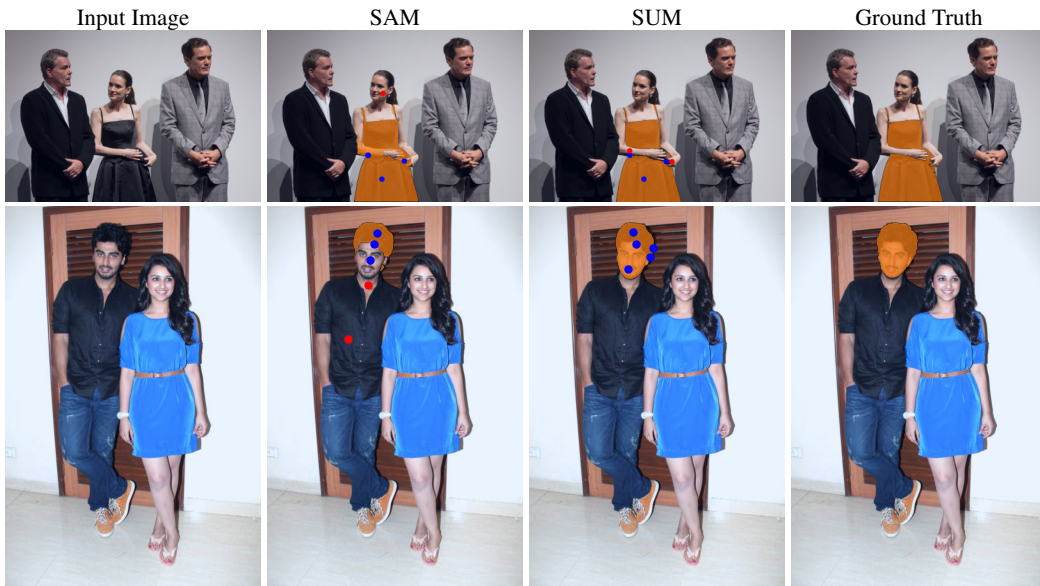

Figure 39: Comparative visualization of segmentation outcomes using point prompts, where blue points signify positive prompts and red points indicate negative prompts. SUM produces predictions of substantially higher quality than those of SAM.

## H.3  SUM versus HQ-SAM

In this section we show a comparison between HQ-SAM [4] and a version of SUM that is constrained to utilize the same lightweight fine-tuning scheme for a fair comparison. We compare (1) the proposed SUM method fine-tuned on FT-Small (i.e. HQ-Seg-44k and SA-250k); (2) The original HQ-SAM, fine-tuned following the Human-Annotation-Only baseline approach (i.e. fine-tuning exclusively on HQ-Seg-44k). Fig. 40, Fig. 43, and Fig. 42 show different examples. Additionally, we show several examples for the part segmentation task in Fig. 43.

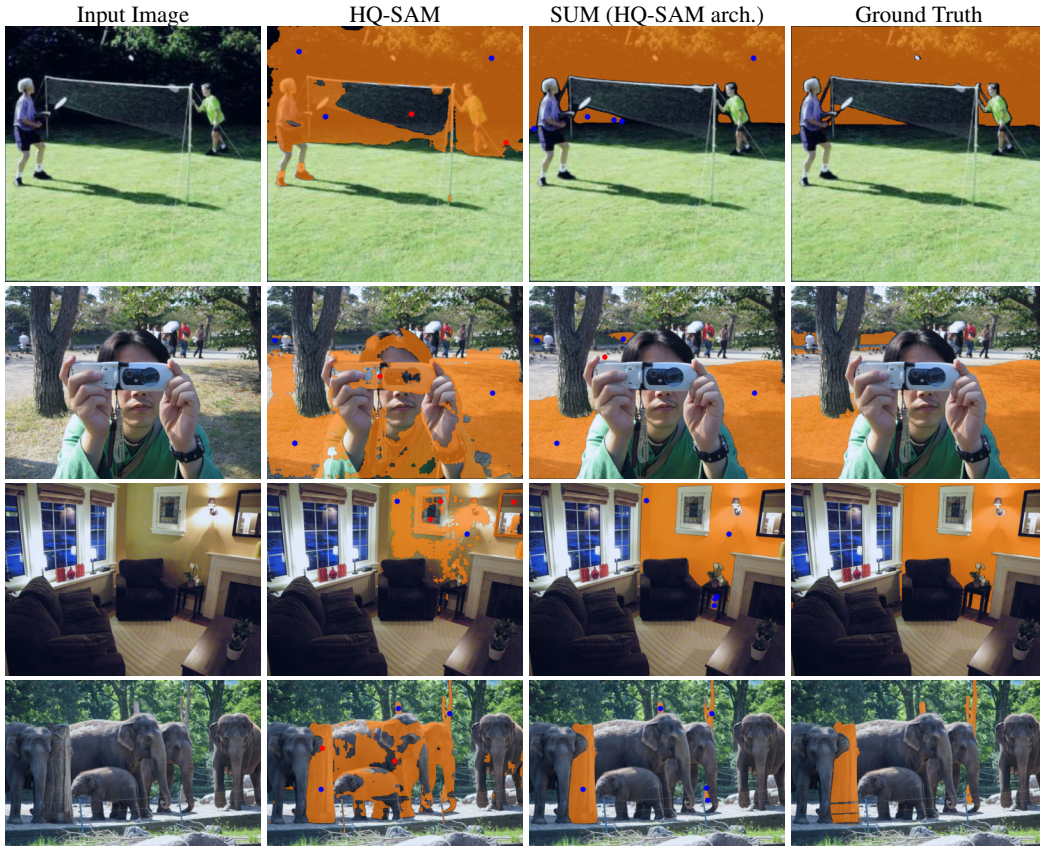

Figure 40: Comparative visualization of entity segmentation outcomes using 5-point prompts, where blue points signify positive prompts and red points indicate negative prompts. SUM achieves better entity segmentation performance than HQ-SAM.

| Input Image | HQ-SAM | SUM (HQ-SAM arch.) | Ground Truth |

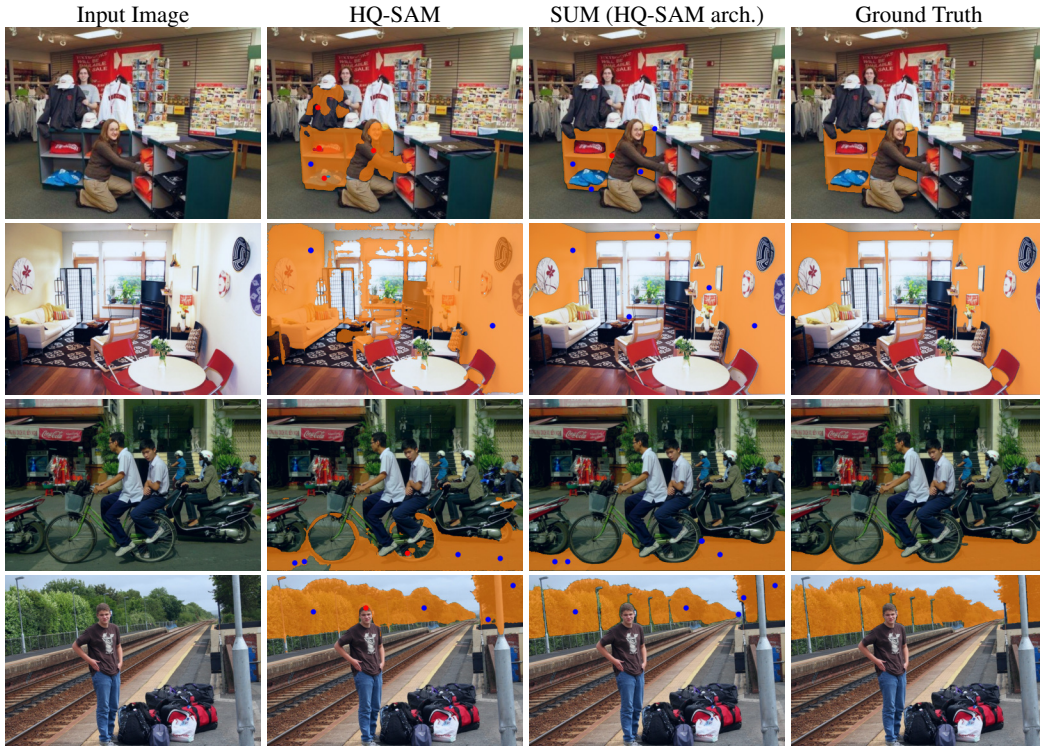

Figure 41: Comparative visualization of entity segmentation outcomes using 5-point prompts, where blue points signify positive prompts and red points indicate negative prompts. SUM achieves better entity segmentation performance than HQ-SAM.

| Input Image | HQ-SAM | SUM (HQ-SAM arch.) | Ground Truth |

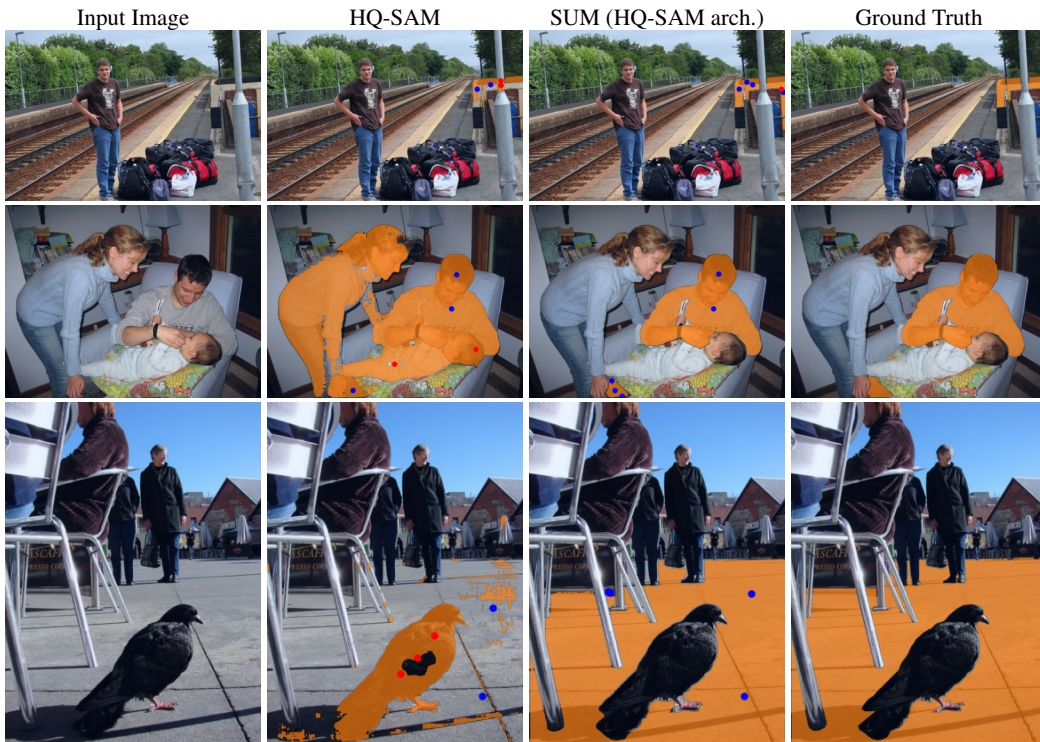

Figure 42: Comparative visualization of entity segmentation outcomes using 5-point prompts, where blue points signify positive prompts and red points indicate negative prompts. SUM achieves better entity segmentation performance than HQ-SAM.

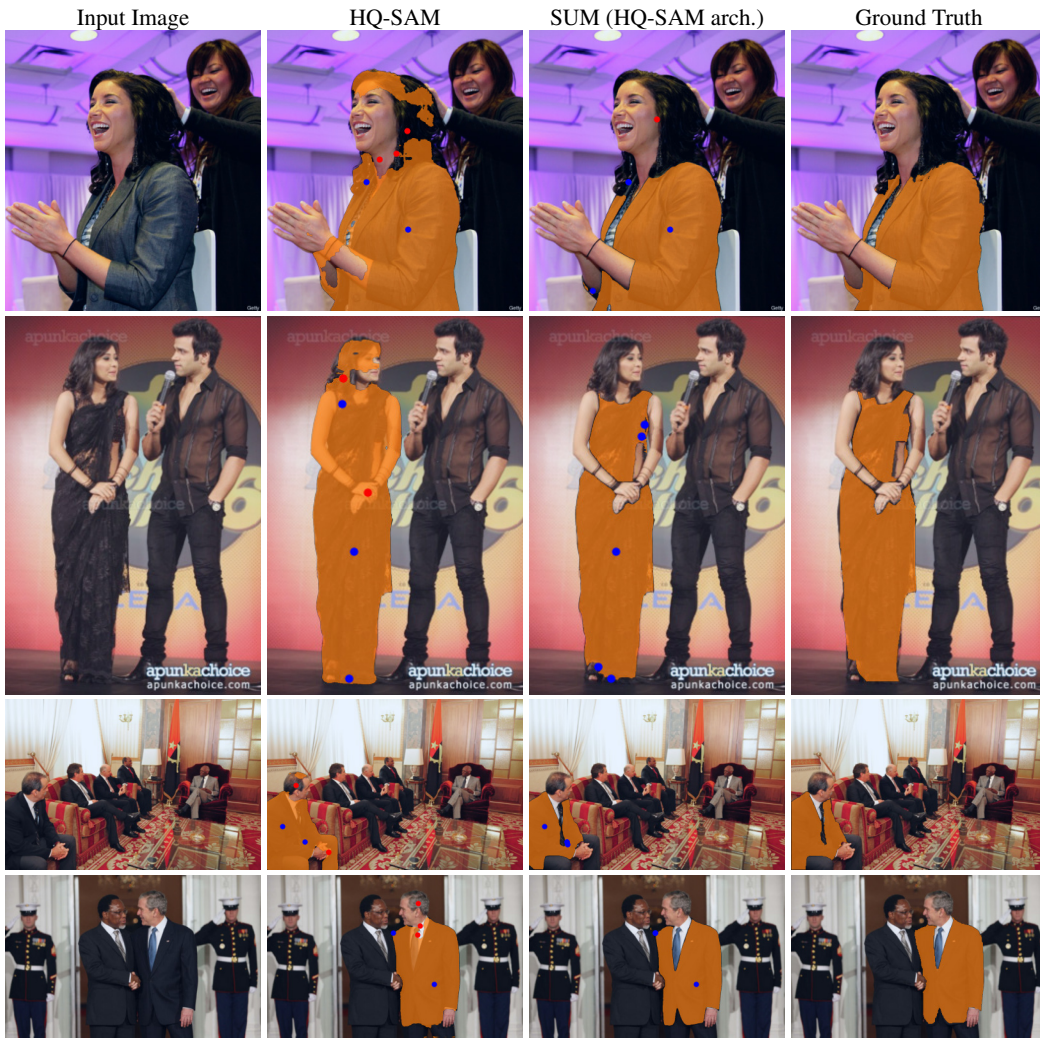

Figure 43: Comparative visualization of part segmentation outcomes using 5-point prompts, where blue points signify positive prompts and red points indicate negative prompts. SUM achieves better part segmentation performance than HQ-SAM.

# I Human Evaluation

## I.1 SAM Pseudo Label Quality

In this section, we report a comprehensive human evaluation on a subset of the SA-1B dataset, consisting of 20,000 randomly selected masks. The evaluation protocol involves a series of questions designed to assess the quality of the segmentation masks in terms of coverage, boundary precision, and object part identification.

The criteria for evaluation are defined as follows:

- **Coverage**: This criterion determines whether a mask accurately encompasses the target object or region within an image. It is deemed 'correct' if the mask captures the entire object or region without fragmentation or inclusion of adjacent non-target entities. While exact adherence to the object's edges is not mandatory, the mask must include all components of the object or specified area. For objects that are part of a larger entity, correct coverage is achieved when the mask fully encloses the object's subset. The coverage is labeled as 'undecided' if the object's boundaries are indiscernible, such as in conditions of poor visibility or when edges are occluded.

- **Boundary**: This aspect examines how well a mask's edges align with the actual boundary of the target entity. A boundary is considered 'accurate' if the mask's periphery matches the entity's contours throughout. Conversely, a boundary is 'inaccurate' if there is a noticeable discrepancy between the mask's edges and the entity's true outline, irrespective of overall coverage adequacy. Similar to coverage, the boundary is rated as 'undecided' when the true edges of the entity are not clearly distinguishable, which may occur in conditions of low visibility or obstruction.

- **Part**: Human participants respond with 'yes' or 'no' to determine if the masked region represents a portion of another object. The term 'part' specifically refers to a segmented region that is a constituent of a larger object. In contrast, standalone entities such as a person, animal, or car are not considered 'parts.' Similar to the previous two categories, we also provide 'undecided' option in case it is not clearly distinguishable.

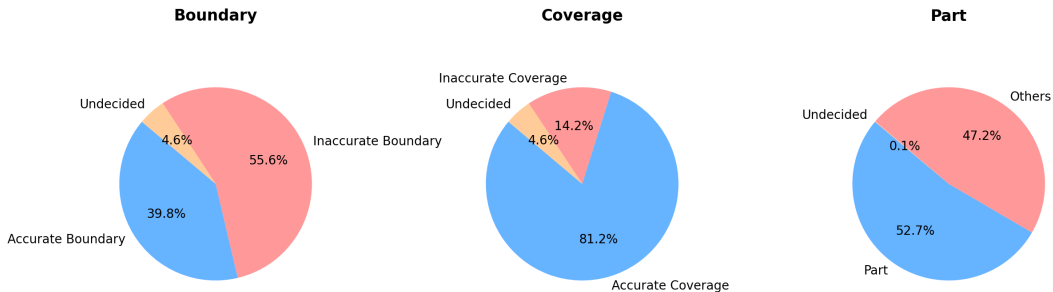

Figure 44: **Human evaluation of SAM pseudo label quality.** We conducted a human evaluation on randomly selected 20,000 masks in the SA-1B dataset. While the overall coverage of the object masks is satisfactory, the boundaries are predominantly of poor quality.

Fig. 44 shows some key findings:

1. Approximately 55.6% of the pseudo labels were identified with boundary inaccuracies.
2. About 81.2% of the labels were adjudged to have accurate coverage.
3. 52.7% were classified as part masks.

This analysis yields valuable insights, indicating that while the overall coverage of the object masks is satisfactory, the boundaries are predominantly of poor quality. Consequently, refining masks emerges as a viable approach to enhance mask quality without necessitating significant alterations to the original pseudo labels in many cases.

In addition, we studied the sample filtering metrics used in SAM. SAM utilizes two key metrics, Predicted IoU and Stability Score, for pseudo label filtering. For a more detailed description of the

two metrics, please refer to Appendix B (automatic mask generation details) in [1]. Here, our analysis is specifically focused on examining the role of the filtering metrics in further discerning the quality of pseudo labels that have already been subjected to an initial filtering phase.

Fig. 45 and Fig. 46 show the results. For boundary accuracy, the Predicted IoU did not show significant variance between accurate and inaccurate classifications (p-value = 0.548), unlike the Stability Score (p-value = 0.00087). In terms of coverage accuracy, both Predicted IoU (p-value = 0.037) and Stability Score (p-value = 5.83e-06) displayed significant differences between correct and incorrect categorizations. However, the observed overlap in both metrics across quality categories suggests limitations in their discriminative ability to accurately assess boundary accuracy and coverage after the initial filtering.

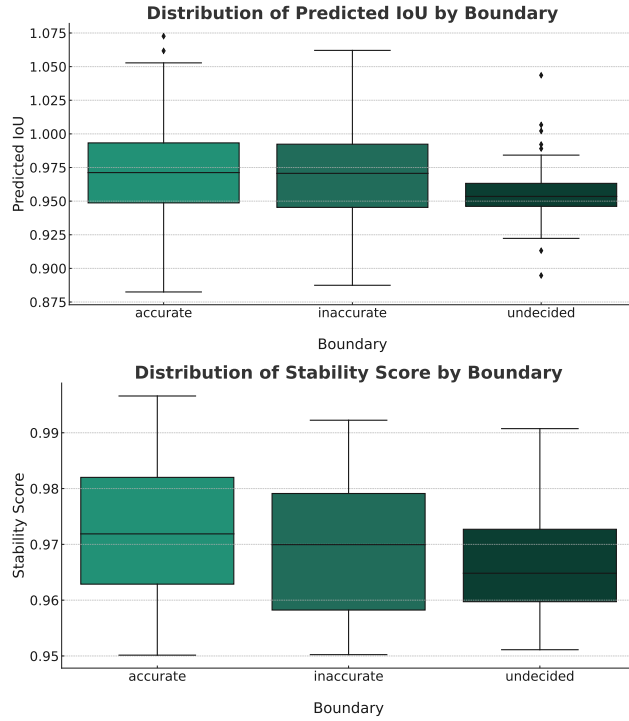

Figure 45: **Top**: Distribution of Predicted Intersection over Union (IoU) by Boundary Accuracy. This boxplot illustrates the distribution of predicted IoU values across images categorized by the boundary accuracy. The categories include accurate, inaccurate, and undecided boundaries. The median, quartiles, and outliers are represented within each category, indicating the variability of the IoU metric in relation to boundary accuracy. **Bottom**: The results for the Stability Score are similar.

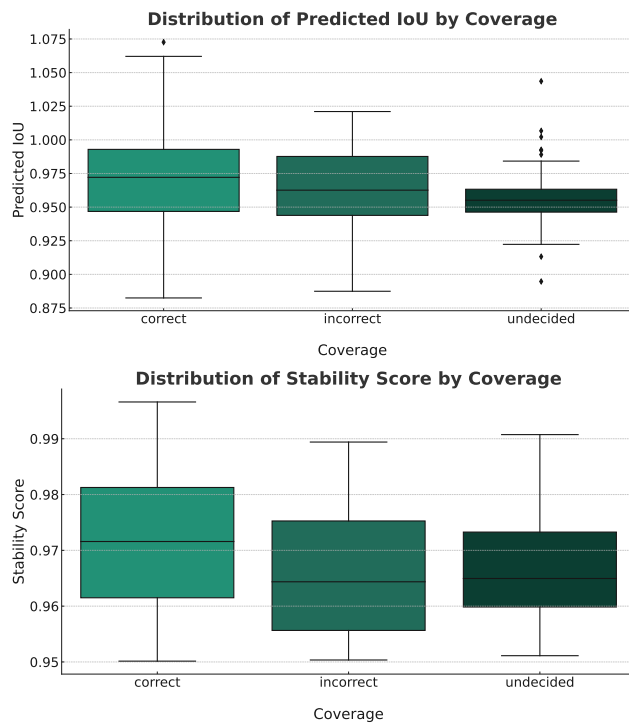

Figure 46: **Top:** Distribution of Predicted Intersection over Union (IoU) by Coverage Correctness. Presented here are the predicted IoU values for images, grouped by the correctness of coverage. The categories include correct, incorrect, and undecided coverage. **Bottom:** Distribution of Stability Scores by Coverage Correctness. This boxplot showcases stability scores across images with varying levels of coverage correctness.

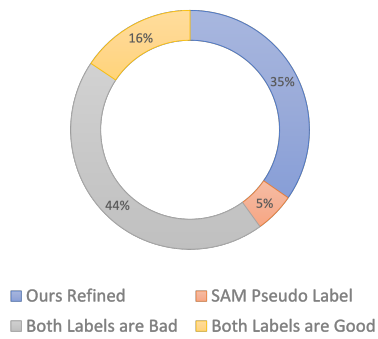

Comparison of Boundary Accuracy

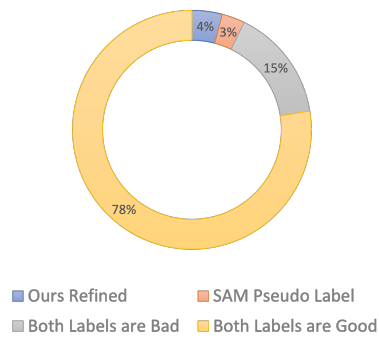

Comparison of Coverage

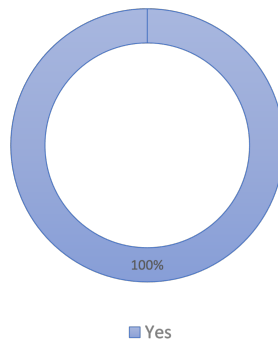

Semantic Correspondence

Figure 47: This figure presents the outcomes of a human evaluation comparing segmentation masks. A total of 2,000 mask pairs, randomly selected from the SA-1B dataset, were assessed for boundary accuracy and coverage. Participants were tasked with determining the accuracy of mask boundaries and the adequacy of target area coverage, choosing between the SAM Pseudo Label and Ours Refined for each metric, or indicating if both labels were satisfactory or unsatisfactory. The results demonstrate a clear preference for masks refined by our module, highlighting improvements in boundary quality without compromising the comprehensive coverage of the original SAM masks. Semantic correspondence between the original and refined masks was also verified.

## I.2 Mask-refinement Module

We tested whether the proposed mask-refinement module improves the quality of SAM pseudo labels via human evaluation. Participants were presented with the task of comparing segmentation masks generated by the SAM pseudo label approach against those refined by our mask-refinement module. We randomly selected 2,000 mask pairs from the SA-1B dataset to undergo comparison. To ensure a fair comparison, we randomized the listing order of the two segmentation masks for each pair. The evaluation was focused on two key quality metrics:

- **Boundary Accuracy**: Participants were asked to assess the accuracy of the mask's boundary in relation to the true outline of the target object. The options provided for the judgment included: (i) SAM Pseudo Label, indicating that the original SAM-generated mask had more accurate boundaries, (ii) Ours Refined, denoting more accurate boundary of our refined mask, and (iii) Both Labels are Bad or Both Labels are Good, to reflect situations where neither or both approaches were satisfactory.

- **Coverage**: For the coverage metric, participants evaluated the extent to which the mask encompasses the intended target area without including extraneous regions or omitting portions of the object. The same response options were provided as for the boundary accuracy assessment.

- **Semantic Correspondence**: Participants reviewed whether the masks align in terms of their semantic categories. For instance, a part mask in one should correspond to a part mask in the other; similarly, an instance mask should not encompass multiple entities if its counterpart does not.

Fig. 47 presents the human evaluation results, which demonstrate the improved accuracy in boundary quality of our refined masks over the original SAM pseudo labels. Our approach also preserves the extensive coverage of the SAM model's masks. Additionally, there is a confirmed semantic correspondence between the pseudo and refined masks.

